# How does PDE order affect the convergence of PINNs?

**Changhoon Song**[1]*    **Yesom Park**[1,2]*    **Myungjoo Kang**[3]

[1] Research Institute of Mathematics, Seoul National University
[2] Department of Mathematics, University of California, Los Angeles
[3] Department of Mathematical Sciences, Seoul National University
changhoon.song93@gmail.com, yeisom@math.ucla.edu, mkang@snu.ac.kr

## Abstract

This paper analyzes the inverse relationship between the order of partial differential equations (PDEs) and the convergence of gradient descent in physics-informed neural networks (PINNs) with the power of ReLU activation. The integration of the PDE into a loss function endows PINNs with a distinctive feature to require computing derivatives of model up to the PDE order. Although it has been empirically observed that PINNs encounter difficulties in convergence when dealing with high-order or high-dimensional PDEs, a comprehensive theoretical understanding of this issue remains elusive. This paper offers theoretical support for this pathological behavior by demonstrating that the gradient flow converges in a lower probability when the PDE order is higher. In addition, we show that PINNs struggle to address high-dimensional problems because the influence of dimensionality on convergence is exacerbated with increasing PDE order. To address the pathology, we use the insights garnered to consider variable splitting that decomposes the high-order PDE into a system of lower-order PDEs. We prove that by reducing the differential order, the gradient flow of variable splitting is more likely to converge to the global optimum. Furthermore, we present numerical experiments in support of our theoretical claims.

## 1 Introduction

Understanding of partial differential equations (PDEs) is fundamental in describing diverse phenomena in science and engineering, including fluid dynamics [60, 17], weather prediction [51], disease progression [3, 46], and quantum mechanics [20, 8]. This underscores the imperative necessity for the effective acquisition of their solutions. Given that analytically solving PDEs is often infeasible or even impossible for numerous practical scenarios due to their complexity, numerical methodologies play a pivotal role in approximating solutions to PDEs, enabling researchers and engineers to address real-world problems effectively.

The advent of deep learning has led to a surge in attempts to leverage it to solve PDEs [59, 42, 30]. Among these, physics-informed neural networks (PINNs) [39, 21, 38, 58] stand out as a prominent methodology. Coupled with the automatic differentiation technique [7], they integrate the residuals of PDEs and boundary conditions into the loss function, thereby enforcing the approximation of solutions using artificial neural networks. This distinctive incorporation of PDEs into the loss function introduces partial differential operators in calculating the loss, distinguishing PINNs from conventional deep learning models. Renowned for their accessibility and versatility in being capable of easily handling arbitrary PDEs and being mesh-free, PINNs have garnered significant attention and demonstrated promising outcomes across various fields [13, 28, 2, 65].

Despite their potential, PINNs frequently encounter difficulties in accurately approximating solutions, particularly when the governing PDE contains high-order derivatives [48, 33]. They also exhibit sensitivity to increasing dimensions [33]. These challenges impede the practicality of PINNs due to the pervasiveness of high-order or high-dimensional PDEs in numerous physical and engineering descriptions, such as control problems [22, 57], finance [5, 53], phase separation [12, 27], and mechanical engineering [4, 32]. Several studies have indicated that neural network architectures possess sufficient expressive power to approximate solutions [31, 41]. However, it has been purported that the inferior performance may be attributed to the difficulty in optimizing PINNs, which arises from including the PDE in the loss function [37, 61, 62]. Despite the widespread use of PINNs, a rigorous mathematical understanding of these pathological behaviors of PINNs has been lacking.

In this paper, we endeavor to provide a mathematical understanding of the pathological behaviors of PINNs by analyzing the convergence of their gradient flow (GF), which reveals a profound sensitivity of the GF convergence with respect to the PDE order and the power of the activation. Building upon the work of Gao et al. [25], we extend the analysis of the GF of PINNs, composed of two-layer multilayer perceptrons (MLPs), to general $k$th-order PDEs and the $p$-th power of Rectified Linear Unit (ReLU) activation function with general $p$. We achieve tighter bounds than those obtained by Gao et al., shedding light on the underlying causes of the pathological behaviors of PINNs. Our theoretical findings demonstrate that the width size of the network necessary for the convergence of the GF increases exponentially with the power $p$ of ReLU$^p$ activation. Furthermore, our results indicate that the optimal power $p$ is determined by the order $k$ of the governing PDE, specifically to be $k + 1$. We also find that the PDE order impedes the convergence of GF, where this negative impact of the PDE order stems from incorporating the PDE into the PINN loss function, which necessitates network differentiation up to the order of the PDE. Moreover, our theoretical investigation unveils that the GF convergence of PINNs also deteriorates with increasing dimensions, and the differential operators included in the PINN loss further exacerbate the sensitivity of PINNs to dimensionality. This elucidates why PINNs are relatively sensitive to dimensionality compared to conventional deep learning models that do not involve differentiation in the loss function.

To address these challenges, we mathematically demonstrate the efficacy of a variable splitting strategy [54, 55, 6], which represents derivatives of the solution as additional auxiliary variables. The key point of variable splitting is that learning a high-order PDE boils down to learning a system of lower-order PDEs. Reducing the order of derivatives included in the loss function, the strategy alleviates the difficulties associated with the PDE order. It further enables to utilize more general ReLU$^p$ activation with lower power $p$ than PINNs. The lower differential orders that the network computes, the more likely it is that the GF will converge, so the most suitable one among the various constructions of the variable splitting method is the finest splitting, which separates all the derivatives into auxiliary variables and reformulates the PDE into a system of first-order PDEs. This strategy results in a loss function comprising only first-order derivatives, and the efficacy of this finest variable splitting would be magnified as the order of the governing PDE or dimension increases. Therefore, the finest splitting approach would exhibit a pronounced discrepancy from the vanilla PINNs for high-order PDEs. Moreover, a reduction in the differential orders enhances the resilience of the model with respect to dimensionality. Finally, we present numerical experiments to verify our theoretical findings and validate the effectiveness of the variable splitting.

## 1.1 Related Work

**Characterization of Gradient Descent for PINNs**    As significant issues have been identified within physics-informed machine learning, numerous mathematical studies have been conducted to elucidate the behavior of PINNs. While studies have been mainly dedicated to examining the generalization capacity of PINNs [19, 49, 23], there has also been work on understanding the difficulty of optimization, which is believed to be the primary source of failure for PINNs. Wang et al. [61] found that PINNs exhibit stiff gradient flow dynamics, resulting in imbalanced gradients during training. Ryck et al. [18] characterized the rate of convergence in terms of the conditioning of an operator and suggested that the difficulty of training PINNs is closely related to the conditioning of the differential operators in the governing PDEs. Another work [62] utilized the neural tangent kernel (NTK) theory to indicate that spectral biases and discrepancies between convergence rates of various loss components can lead to training instabilities. Global convergence properties of PINNs for second-order linear PDEs have also been studied within the NTK regime [34] and using the Rademacher complexity [47]. Most closely related to this paper, Gao et al. [25] demonstrated the

convergence of the gradient descent for two-layer PINNs. However, their discussion is limited to second-order linear PDEs. We extend the analysis to general $k$th-order linear PDEs and $p$-th power of activation functions and provide tighter bounds than Gao et al.. These advances allow us to observe further the effect of the order and dimensionality of the PDE on the convergence.

**Variable Splitting**  The method of separation of variables, which simplifies differential equations by reformulating them into a more manageable system, is a classical method for solving differential equations [9]. In particular, it has been widely employed when dealing with high-order PDEs as augmenting high-order derivatives as additional variables allows the governing equation to be decoupled into a set of lower-order PDEs that are comparatively easy to solve [16, 26, 63]. Recent endeavors have explored the integration of separable variables within the PINN approach. In this paper, we refer to this approach as variable splitting according to [54, 55, 56]. Augmented variables have been introduced to represent vorticity in the Stokes equation [6], the gradient of the solutions for solving the $p$-Poisson equation [54], and the eikonal equation [55]. Additionally, second-order derivatives have been separately parameterized to solve bi-harmonic equations effectively [45]. The rationale for introducing auxiliary variables in previous works is to enhance the efficiency and accuracy of PINNs, but they lack a comprehensive theoretical elucidation of its effect. A recent study [56] has theoretically analyzed variable splitting, demonstrating that while PINNs do not guarantee convergence to the PDE solution even when the loss converges to zero, variable splitting does ensure convergence to the solution for second-order linear PDEs. In this study, we analyze the impact of variable splitting for PINNs with ReLU$^p$ activation in terms of the convergence of the GF.

## 1.2  Main Contributions

The contribution of the paper is summarized as follows.

- We analyze that the GF of PINNs with ReLU$^p$ activation converges to the global minimum for general $k$th-order linear PDEs. This extends the findings of Gao et al. [25] to encompass a broader range of PDEs and activations and provides an even tighter bound.

- We demonstrate the inverse relation between PDE order and the GF convergence, unveiling the adverse effect of the differentials included in the PINN loss on the GF convergence.

- We provide a theoretical understanding of the reasons why PINNs encounter difficulties in addressing high-dimensional problems.

- We prove that the order reduction of variable splitting, which reformulates the PDEs into a system of lower-order PDEs, results in the convergence enhancement of GF.

## 2  Mathematical Setup

**Arbitrary Order Linear PDEs**  We consider a general form of *$k$th-order linear partial differential equations (PDEs)* defined on a bounded domain $\Omega \subset \mathbb{R}^d$ (in which the temporal dimension could be a subcomponent)

$$\begin{cases} \mathcal{N}[u](\boldsymbol{x}) = f(\boldsymbol{x}), & \boldsymbol{x} \in \Omega, \\ \mathcal{B}[u](\boldsymbol{x}) = g(\boldsymbol{x}), & \boldsymbol{x} \in \partial\Omega, \end{cases} \tag{1}$$

where $\mathcal{N}[u] = \sum_{|\alpha| \leq k} a_\alpha \frac{\partial^\alpha}{\partial \boldsymbol{x}^\alpha} u$ is a $k$th-order linear differential operator with coefficient functions $a_\alpha : \Omega \to \mathbb{R}$ for each multi-index $\alpha \in \mathbb{N}_0^d$, $\mathcal{B}[u] = \sum_{|\alpha| \leq 1} \tilde{a}_\alpha \frac{\partial^\alpha}{\partial \boldsymbol{x}^\alpha} u$ represents the boundary condition operator with coefficient functions $\tilde{a}_\alpha : \partial\Omega \to \mathbb{R}$, which could reflect Dirichlet, Neumann, and Robin conditions [2], $f : \Omega \to \mathbb{R}$ is a given source function, and $g : \partial\Omega \to \mathbb{R}$ is a given boundary function, and $u : \bar{\Omega} \to \mathbb{R}$ is the unknown solution of interest.

**Physics-InformedNeural Networks**   Physics-informed neural networks (PINNs) [58] aim to approximate the solution $u$ of the PDE by neural networks. Following the prior work [25], we approximate the solution $u$ by a two-layer multi-layer perceptron $\phi : \mathbb{R}^d \to \mathbb{R}$ of width $m$, defined as

$$\phi\left(\boldsymbol{x}; \boldsymbol{w}, \boldsymbol{v}\right) = \frac{1}{\sqrt{m}} \sum_{r=1}^{m} v_r \cdot \sigma\left(\boldsymbol{w}_r^\top \boldsymbol{y}\right), \tag{2}$$

where $\boldsymbol{w}_r \in \mathbb{R}^{d+2}$, $v_r \in \mathbb{R}$, $\boldsymbol{w} = \left[\boldsymbol{w}_0^\top, \cdots \boldsymbol{w}_m^\top\right]^\top \in \mathbb{R}^{m(d+2)}$ $\boldsymbol{v} = \left[\boldsymbol{v}_0^\top, \cdots \boldsymbol{v}_m^\top\right]^\top \in \mathbb{R}^m$, $\boldsymbol{y} = \left[\boldsymbol{x}^\top \quad \frac{1}{2}\right]^\top \in \mathbb{R}^{d+1}$, and $\sigma\left(\cdot\right)$ is the activation function. For brief notations, we assume that $\Omega$ is bounded so that $\|\boldsymbol{y}\|_2 \leq 1$ for $\boldsymbol{x} \in \Omega$. We consider the case where $\sigma$ is the ReLU$^p$ activation function for an integer $p$, which is also known as Rectified Power(RePU) activation [1, 10, 15]. As it will be clear in the context, the power $p$ necessitates surpassing the order $k$ of the PDE (1) to ensure that the loss function and gradient descent flow are well-defined. Therefore, our analysis is focused on scenarios where $p \geq k+1$. PINNs learn the parameters of $\phi$ by minimizing a composite loss function, comprising the residual of the PDE and the boundary condition of (1), which enforces the network's compliance with the governing physics. For given the training data $\{\boldsymbol{x}_i, f\left(\boldsymbol{x}_i\right)\}_{i=1}^{n_o} \subset \Omega \times \mathbb{R}$ and $\{\tilde{\boldsymbol{x}}_j, g\left(\tilde{\boldsymbol{x}}_j\right)\}_{j=1}^{n_b} \subset \partial\Omega \times \mathbb{R}$ of respective sizes $n_o \in \mathbb{N}$ and $n_b \in \mathbb{N}$, PINN loss function is given by

$$\mathcal{L}_{PINN}\left(\boldsymbol{w}, \boldsymbol{v}\right) = \frac{1}{2}\left(\|\boldsymbol{s}\left(\boldsymbol{w}, \boldsymbol{v}\right)\|^2 + \|\boldsymbol{h}\left(\boldsymbol{w}, \boldsymbol{v}\right)\|^2\right), \tag{3}$$

where $\boldsymbol{s}\left(\boldsymbol{w}, \boldsymbol{v}\right) = \left[s_1\left(\boldsymbol{w}, \boldsymbol{v}\right) \quad \cdots s_{n_o}\left(\boldsymbol{w}, \boldsymbol{v}\right)\right]^\top$ and $\boldsymbol{h}\left(\boldsymbol{w}, \boldsymbol{v}\right) = \left[h_1\left(\boldsymbol{w}, \boldsymbol{v}\right) \quad \cdots h_{n_b}\left(\boldsymbol{w}, \boldsymbol{v}\right)\right]^\top$ with

$$s_i\left(\boldsymbol{w}, \boldsymbol{v}\right) = \sqrt{\frac{1}{n_o}}\left(\mathcal{N}\left[\phi\left(\cdot; \boldsymbol{w}, \boldsymbol{v}\right)\right]\left(\boldsymbol{x}_i\right) - f\left(\boldsymbol{x}_i\right)\right), \tag{4}$$

$$h_j\left(\boldsymbol{w}, \boldsymbol{v}\right) = \sqrt{\frac{\nu}{n_b}}\left(\mathcal{B}\left[\phi\left(\cdot; \boldsymbol{w}, \boldsymbol{v}\right)\right]\left(\tilde{\boldsymbol{x}}_j\right) - g\left(\tilde{\boldsymbol{x}}_j\right)\right), \tag{5}$$

and $\nu > 0$ is a regularization parameter that relatively balances the two components of the loss.

**Gradient Flow**   As the limiting dynamics of the gradient descent (GD) with infinitesimal step-sizes [40], gradient flow (GF) is continuous time dynamics that starts at $\boldsymbol{w}\left(0\right)$ and $\boldsymbol{v}\left(0\right)$ and evolves as

$$\begin{cases} \frac{d\boldsymbol{w}_r(t)}{dt} = -\frac{\partial \mathcal{L}_{PINN}(\boldsymbol{w}, \boldsymbol{v})}{\partial \boldsymbol{w}_r} = -\sum_{i=1}^{n_o} s_i\left(\boldsymbol{w}, \boldsymbol{v}\right) \cdot \frac{\partial s_i(\boldsymbol{w}, \boldsymbol{v})}{\partial \boldsymbol{w}_r} - \sum_{j=1}^{n_b} h_j\left(\boldsymbol{w}, \boldsymbol{v}\right) \cdot \frac{\partial h_j(\boldsymbol{w}, \boldsymbol{v})}{\partial \boldsymbol{w}_r}, \\ \frac{d v_r(t)}{dt} = -\frac{\partial \mathcal{L}_{PINN}(\boldsymbol{w}, \boldsymbol{v})}{\partial v_r} = -\sum_{i=1}^{n_o} s_i\left(\boldsymbol{w}, \boldsymbol{v}\right) \cdot \frac{\partial s_i(\boldsymbol{w}, \boldsymbol{v})}{\partial v_r} - \sum_{j=1}^{n_b} h_j\left(\boldsymbol{w}, \boldsymbol{v}\right) \cdot \frac{\partial h_j(\boldsymbol{w}, \boldsymbol{v})}{\partial v_r}. \end{cases} \tag{6}$$

Initial weights are supposed to follow the normal and uniform distributions, $\boldsymbol{w}\left(0\right) \sim N\left(0, I_m\right)$ and $\boldsymbol{v} \sim U\left(\{-1, 1\}\right)$, respectively[3]. GF can be regarded as a continuous-time analog of GD and is frequently employed to comprehend the behavior of GD optimization algorithm in the limit. By the chain rule in conjunction with (6), the following characterizes how the loss function evolves during training by gradient descent:

$$\frac{d}{dt}\begin{bmatrix} \boldsymbol{s}\left(\boldsymbol{w}\left(t\right), \boldsymbol{v}\left(t\right)\right) \\ \boldsymbol{h}\left(\boldsymbol{w}\left(t\right), \boldsymbol{v}\left(t\right)\right) \end{bmatrix} = -\left(\boldsymbol{G}_{\boldsymbol{w}}\left(\boldsymbol{w}\left(t\right), \boldsymbol{v}\left(t\right)\right) + \boldsymbol{G}_{\boldsymbol{v}}\left(\boldsymbol{w}\left(t\right), \boldsymbol{v}\left(t\right)\right)\right)\begin{bmatrix} \boldsymbol{s}\left(\boldsymbol{w}\left(t\right), \boldsymbol{v}\left(t\right)\right) \\ \boldsymbol{h}\left(\boldsymbol{w}\left(t\right), \boldsymbol{v}\left(t\right)\right) \end{bmatrix}, \tag{7}$$

where $\boldsymbol{G}$ and $\tilde{\boldsymbol{G}}$ are Gram matrices for the dynamics, defined by

$$\boldsymbol{G}_{\boldsymbol{w}}\left(\boldsymbol{w}, \boldsymbol{v}\right) = \boldsymbol{D}_{\boldsymbol{w}}^\top \boldsymbol{D}_{\boldsymbol{w}}, \boldsymbol{D}_{\boldsymbol{w}} = \begin{bmatrix} \frac{\partial s_1}{\partial \boldsymbol{w}}\left(\boldsymbol{w}, \boldsymbol{v}\right) & \cdots & \frac{\partial s_{n_o}}{\partial \boldsymbol{w}}\left(\boldsymbol{w}, \boldsymbol{v}\right) & \frac{\partial h_1}{\partial \boldsymbol{w}}\left(\boldsymbol{w}, \boldsymbol{v}\right) & \cdots & \frac{\partial h_{n_b}}{\partial \boldsymbol{w}}\left(\boldsymbol{w}, \boldsymbol{v}\right) \end{bmatrix} \tag{8}$$

$$\boldsymbol{G}_{\boldsymbol{v}}\left(\boldsymbol{w}, \boldsymbol{v}\right) = \boldsymbol{D}_{\boldsymbol{v}}^\top \boldsymbol{D}_{\boldsymbol{v}}, \boldsymbol{D}_{\boldsymbol{v}} = \begin{bmatrix} \frac{\partial s_1}{\partial \boldsymbol{v}}\left(\boldsymbol{w}, \boldsymbol{v}\right) & \cdots & \frac{\partial s_{n_o}}{\partial \boldsymbol{v}}\left(\boldsymbol{w}, \boldsymbol{v}\right) & \frac{\partial h_1}{\partial \boldsymbol{v}}\left(\boldsymbol{w}, \boldsymbol{v}\right) & \cdots & \frac{\partial h_{n_b}}{\partial \boldsymbol{v}}\left(\boldsymbol{w}, \boldsymbol{v}\right) \end{bmatrix}. \tag{9}$$

We are interested in analyzing the effect of the PDE order on the convergence of the PINN loss, which evolved in accordance with the dynamics (7), to the global minimum zero.

# 3 Impact of PDE Order on Convergence of PINNs

Despite the demonstrated promise and versatility of PINNs in addressing a wide range of problems [29, 44, 36], they often encounter difficulties in constructing an accurate approximation to the desired solution of PDEs, particularly with high-order PDEs. Moreover, in contrast to the confirmed efficacy of neural networks in modeling high-dimensional data such as images and text, the exploration of PINNs for high-dimensional PDEs has been apparently limited. While neural network architectures possess sufficient expressive power to approximate solutions [11], inferior performance has been attributed to the difficulty in optimization in practice [61, 62]. Additionally, it has been postulated that the optimization difficulty may stem from the partial differential operators included in the loss function [37, 48, 33]. Nevertheless, despite the significant challenge posed by these pathological phenomena, there remains a paucity of theoretical understanding of them.

In this section, we theoretically elucidate these pathological phenomena by studying the convergence condition of GF (7) of PINN loss (3). Specifically, we provide a width condition for (7) to converge to global optimum in terms of order $k$, dimension $d$, and the power $p$ of ReLU activation. Analyzing how those factors are related to the convergence condition, we explain why optimizing PINNs is harder when the order or degree is higher.

Following [24] and [25], we first prove the positive definiteness of the limiting Gram matrix of PINNs for general $k$th-order linear PDE and $p$ without any further strict assumption other than $p > k$.

**Proposition 3.1** (Special Case)**.** *The limiting Gram matrix* $\boldsymbol{G}_{\boldsymbol{v}}^{\infty} = \mathbb{E}_{\boldsymbol{w},\boldsymbol{v}}\left[\boldsymbol{G}_{\boldsymbol{v}}\left(\boldsymbol{w},\boldsymbol{v}\right)\right]$ *is strictly positive definite and independent of* $m$.

This is a special case of the general state in Proposition C.3 with $L = 0$, and the proof for the general case is provided in Appendix C. We denote the smallest eigenvalue of $\boldsymbol{G}_{\boldsymbol{v}}^{\infty}$ by $\lambda_0 > 0$. The following presents our main theorem in this section, the requisite width size $m$ for the GF of PINN loss to converge to the global minimum with high probability. The result demonstrates that the required width grows exponentially as the PDE order $k$ and the dimension of the domain $d$ increase.

**Theorem 3.2** (Special Case)**.** *There exists a constant $C$, independent of $d$, $k$, and $p$, such that for any $\delta \ll 1$, if*

$$m > C\binom{d+k}{d}^{14} p^{7k+4} 2^{6p}\left(\log \frac{md}{\delta}\right)^{4p} \tag{10}$$

*then with probability of at least $1 - \delta$ over the initialization, we have*

$$\mathcal{L}_{PINN}\left(\boldsymbol{w}\left(t\right),\boldsymbol{v}\left(t\right)\right) \leq \exp\left(-\lambda_0 t\right)\mathcal{L}_{PINN}\left(\boldsymbol{w}\left(0\right),\boldsymbol{v}\left(0\right)\right), \ \forall t \geq 0. \tag{11}$$

It is a special case of Theorem 4.3 with $L = 0$, the proof of which can be found in Appendix C.1. It extends, inspired by [25], the convergence of the GF of PINNs of second-order linear PDEs to $k$th-order linear PDEs and the general $p$-th power of ReLU. It states that even in these general settings, the GF of PINNs converges to the global minimum with a high probability when the width of the network is sufficiently large. Moreover, we obtain a polylogarithmic bound $\left(\log \frac{1}{\delta}\right)^{4p}$, which is much tighter than polynomial bound $\delta^{-3}$ in [25] for $p = 3$. These improvements permit the derivation of the following valuable explanations for the deficiencies observed when optimizing PINNs.

**Optimal Power of ReLU Function in Training** Theorem 3.2 sheds light on the suitable choice of activation function for PINNs. In the training process, the activation function plays an important role. However, there are no clues as to which activation function is favorable to the given optimization process. Especially in the case of PINNs, it depends heavily on the PDE at hand. Despite its pervasive use in deep learning due to its numerous advantages and performance benefits, the ReLU activation function is not admissible in the PINN framework, which necessitates the activation function to provide high-order derivatives for optimizing PDE-based constraints. Instead, PINNs harness the $p$-th power of ReLU as the activation function. It is apparent that $p$ must satisfy $p \geq k + 1$ for the PINN loss and gradient descent to be computed. Theorem 3.2 indicates that the smaller $p$ is, the more likely the gradient descent will converge; that is, it is most optimal[4] to adjust $p$ to $k + 1$ regarding the training process.

**Understanding Difficulty in High-order PDEs**   A significant observation of Theorem 3.2 is that it provides a theoretical understanding of why PINNs struggle with high-order PDEs. From (10), we can see that the bound increases exponentially with the order of the PDE. Moreover, for the GF of PINNs to converge with high probability, that is, $\delta \ll 1$, a small increment of the power $p$ would contribute to non-negligible degradation in the convergence, which could ultimately prevent the network from reaching a minimizer of the loss. Hence, given that $k$ determines the admissible $p$ by $p \geq k+1$, the order $k$ of PDE primarily influences the convergence of PINNs and increasing the exponential term in (10).

**Understanding Difficulty in High-dimensional Problems**   The above theorem, which shows that the lower bound of $m$ depends on the exponential of $d$, explains why PINNs cannot completely combat the curse of dimensionality. As PINNs are regarded as a versatile method capable of being mesh-free, they have been expected to be free from the curse of dimensionality [58]. However, the GF of PINNs becomes harder to converge as $d$ increases, requiring the network to be wider. Furthermore, it can be observed that the magnitude of change in $d$ is amplified with respect to the exponent of $k$. This explains why PINNs are relatively sensitive to increasing dimensionality in comparison to other deep learning models whose loss functions do not contain derivatives. In other words, the presence of derivatives in the loss makes PINNs sensitive to changes in dimensionality, and the larger $k$ is, the more difficult PINNs are for high-dimensionality.

Combining all crucial observations from our main theorem, we believe that the impact of the PDE order is one of the primary underlying reasons why PINNs often fail to minimize their loss. In light of this theoretical evidence, the next section describes a variable splitting strategy that addresses these pathologies by properly reducing the differential order in the PINN loss function.

# 4   Order Reduction through Variable Splitting

The previous section indicates that the PDE order $k$ significantly affects the width requirement for the GF to converge. Concurrently, for $k$th-order PDEs, it is necessary to increase the ReLU activation to at least the $k+1$ power in order to ensure a well-defined GF for the PINN loss. Consequently, lowering $k$ could potentially lead to better convergence of the GF. In this section, we introduce variable splitting strategy to decrease the differential order by reformulating the given PDE into a system of lower-order PDEs. We then extend Theorem 3.2 to a more general form in Theorem 4.3.

## 4.1   Variable Splitting

The concept of variable splitting [54, 55, 56] is to rewrite a higher-order PDE into a lower-order system, after which the PINN approach is applied to the system. A crucial aspect of the success of such methods is the reduction of the derivative order present in the training loss function.

**Augment Variables**   For $L \geq 0$ and increasing integers $0 = \xi_0 < \xi_1 < \cdots < \xi_{L+1} = k$, variable splitting augment the derivatives of the solution $\frac{\partial^{\xi_1}}{\partial \boldsymbol{x}^{\xi_1}} u, \ldots, \frac{\partial^{\xi_L}}{\partial \boldsymbol{x}^{\xi_L}} u$ as additional auxiliary variables $\phi_1, \ldots, \phi_L$, respectively. For notational simplicity, we abbreviate the integer set $\{1, \cdots, m\}$ for a positive integer $m$ by $[m]$. For $\ell \in [L]$, each term in $\frac{\partial^{\xi_\ell}}{\partial \boldsymbol{x}^{\xi_\ell}}$ corresponds to $\frac{\partial^{\xi_\ell}}{\partial x_1^{\alpha_1} \cdots \partial x_d^{\alpha_d}} u$ for a multi-index $\alpha = (\alpha_1, \ldots, \alpha_d) \in \mathbb{N}_0^d$ with the size $|\alpha| = \sum_{i=1}^d \alpha_i = n$. Therefore, $\phi_\ell$ is a vector-valued function of size $|I_{\xi_\ell}|$ for the index set $I$ defined in (18). We denote the component of $\phi_\ell$ that corresponds to $\frac{\partial^{\xi_\ell}}{\partial \boldsymbol{x}^\alpha}$ by $(\phi_\ell)_\alpha$.

**Reformulate PDE into Lower-order System**   By replacing each of the differential terms $\frac{\partial^{\xi_\ell}}{\partial \boldsymbol{x}^{\xi_\ell}} u$ with the corresponding auxiliary variables $\phi_\ell$, the differential operator $\mathcal{N}$ in (1) can be rewritten as:

$$\mathcal{N}[u] = \sum_{|\alpha| \leq k} a_\alpha \frac{\partial^{|\alpha|}}{\partial \boldsymbol{x}^\alpha} u = \sum_{\ell=0}^{L} \sum_{|\alpha| \leq \xi_\ell} \sum_{|\beta| \leq \Delta\xi_{\ell+1}} \hat{a}_{\ell,\alpha,\beta} \frac{\partial^{\Delta\xi_{\ell+1}}}{\partial \boldsymbol{x}^\beta} (\phi_\ell)_\alpha, \tag{12}$$

---

not that a network can be easily trained. Indeed, our result leads us to the opposite conclusion that large $p$ could be detrimental to convergence from the optimization perspective. In this paper, we refer to 'optimal' as the sense of being likely trained under the mildest condition rather than approximating the solution with the smallest error.

for some coefficient functions $\hat{a}_{\ell,\alpha,\beta} : \Omega \to \mathbb{R}$ and $\Delta\xi_\ell = \xi_\ell - \xi_{\ell-1}$. Since $\phi_\ell$ represents a function that differentiates the PDE solution $\Delta\xi_\ell$-times more than $\phi_{\ell-1}$, the components of two consecutive variables $\phi_{\ell-1}$ and $\phi_\ell$ are governed by

$$\frac{\partial^{\Delta\xi_\ell}}{\partial \boldsymbol{x}^\beta}\left(\phi_{\ell-1}\right)_\alpha(\boldsymbol{x}) = \left(\phi_\ell\right)_{\alpha+\beta}(\boldsymbol{x}),\ \alpha \in I_{\xi_{\ell-1}},\ \beta \in I_{\Delta\xi_\ell}. \tag{13}$$

From these, the PDE (1) can be identically reformulated by the system of lower-order PDEs:

$$\begin{cases} \hat{\mathcal{N}}\left[\phi_0, \cdots, \phi_L\right](\boldsymbol{x}) = f(\boldsymbol{x}), & \boldsymbol{x} \in \Omega, \\ \frac{\partial^\beta}{\partial \boldsymbol{x}^\beta}\left(\phi_{\ell-1}\right)_\alpha(\boldsymbol{x}) = \left(\phi_\ell\right)_{\alpha+\beta}(\boldsymbol{x}), & \boldsymbol{x} \in \Omega,\ \ell \in [L],\ \alpha \in I_{\xi_{\ell-1}},\ \beta \in I_{\Delta\xi_\ell}, \\ \mathcal{B}\left[\phi_0\right](\boldsymbol{x}) = g, & \boldsymbol{x} \in \partial\Omega. \end{cases} \tag{14}$$

It is of paramount importance to note that the maximum differential order of this system of PDEs is the highest difference of derivative order between consecutive auxiliary variables $|\xi| = \max\{\Delta\xi_\ell : \ell \in [L+1]\}$, which is less than $k$. This aspect gives rise to notable ramifications in our analysis of VS-PINNs, which will be discussed in the next subsection.

**Variable Splitting for PINNs**   In this paper, we consider the parameterization of all variables $\phi_\ell$ with two-layer MLPs with ReLU$^p$ activation function, in a manner analogous to that described in Section 2 for PINNs. The weights in the first and second layers of $\phi_\ell$ are denoted by $\boldsymbol{w}_\ell$ and $\boldsymbol{v}_\ell$, respectively. We use $\boldsymbol{w} = \left[\boldsymbol{w}_1^\top \cdots \boldsymbol{w}_L^\top\right]^\top$ and $\boldsymbol{v} = \left[\boldsymbol{v}_1^\top \cdots \boldsymbol{v}_L^\top\right]^\top$ to refer to the respective collections of all weights. Similar to PINNs, *Variable Splitting for PINNs (VS-PINNs)* employ the linear sum of penalized residuals of each term of the induced system of PDEs (14) as the training loss:

$$\begin{aligned} \mathcal{L}_{PINN}^{VS}(\boldsymbol{w}, \boldsymbol{v}) = \frac{1}{2}\Bigg( & \frac{1}{n_o}\sum_{i=1}^{n_o}\left(\hat{\mathcal{N}}\left[\phi_0, \ldots, \phi_L\right](\boldsymbol{x}_i) - f(\boldsymbol{x}_i)\right)^2 \\ & + \frac{\nu_\ell}{n_o}\sum_{\ell=1}^{L}\sum_{|\alpha|\leq\xi_\ell}\sum_{|\beta|\leq\Delta\xi_{\ell+1}}\left(\frac{\partial^\beta}{\partial \boldsymbol{x}^\beta}\left(\phi_{\ell-1}\right)_\alpha(\boldsymbol{x}_i) - \left(\phi_\ell\right)_{\alpha+\beta}(\boldsymbol{x}_i)\right)^2 \\ & + \frac{\nu}{n_b}\left(\mathcal{B}\left[\phi_0\right](\tilde{\boldsymbol{x}}_j) - g(\tilde{\boldsymbol{x}}_j)\right)^2\Bigg), \end{aligned} \tag{15}$$

where $\nu, \nu_1, \ldots, \nu_L$ are regularization parameters. As the GF of $\mathcal{L}_{PINN}$ is characterized by Gram matrices $\boldsymbol{G_w}$ and $\boldsymbol{G_v}$ induced from the gradients of the residuals of each term in (1), the GF of $\mathcal{L}_{PINN}^{VS}$ is characterized by Gram matrices $\widehat{\boldsymbol{G}}_{\boldsymbol{w}}$ and $\widehat{\boldsymbol{G}}_{\boldsymbol{v}}$, which is induced from the gradients of the residuals of each term in (14). Appendix A gives more details for $\widehat{\boldsymbol{G}}_{\boldsymbol{w}}$ and $\widehat{\boldsymbol{G}}_{\boldsymbol{v}}$.

*Remark* 4.1. In order for high-order PDEs with $k > 2$ to be well-posed, it is necessary to have more boundary conditions than those defined by the boundary operator $\mathcal{B}$ in (1). Although our analysis concentrated on $\mathcal{B}$ that reflect only up to first-order derivatives for the sake of simplicity, our theory can also be applied to more general boundary conditions. Furthermore, the high-order boundary conditions $\mathcal{B}$ can also be reformulated using the auxiliary variables used for $\hat{\mathcal{N}}$. In that case, relations (13) should hold on the boundary $\boldsymbol{x} \in \partial\Omega$. As the reduced system (14) with reformulated boundary condition is equivalent to (1), instability issues were not observed in our numerical experiments even in the absence of artificial boundary conditions on the auxiliary variables unlike to grid-based conventional numerical schemes.

### 4.2   Analysis

A key advantage of VS-PINNs is that the derivative order of the induced system of PDEs (14) is $|\xi|$, which is lower than that of the original PDE (1). We prove its effectiveness in this section. As analogous to PINNs, we begin by proving the positive definiteness of the limiting Gram matrix, providing its proof in Appendix C.

**Proposition 4.2** (General Case)**.** *The limiting Gram matrix $\widehat{\boldsymbol{G}}_{\boldsymbol{v}}^\infty = \mathbb{E}_{\boldsymbol{w},\boldsymbol{v}}\left[\widehat{\boldsymbol{G}}_{\boldsymbol{v}}(\boldsymbol{w}, \boldsymbol{v})\right]$ is strictly positive definite and independent of $m$.*

We denote the smallest eigenvalue of $\widehat{\boldsymbol{G}}_{\boldsymbol{v}}^\infty$ by $\lambda_0 > 0$. We now present our main theorem, which demonstrates the profound impact of order reduction in variable splitting. The proof of the following theorem can be found in C.1.

**Theorem 4.3** (General Case). *There exists a constant $C$, independent of $d$, $k$, $|\xi|$, and $p$, such that for any $\delta \ll 1$, if*

$$m > C \binom{d+k}{d}^6 \binom{d+|\xi|}{d}^8 p^{7|\xi|+4} 2^{6p} \left( \log \frac{md}{\delta} \right)^{4p}, \tag{16}$$

*then with probability of at least $1 - \delta$ over the initialization, we have*

$$\mathcal{L}_{PINN}^{VS} \left( \boldsymbol{w}\left( t \right), \boldsymbol{v}\left( t \right) \right) \leq \exp\left( -\lambda_0 t \right) \mathcal{L}_{PINN}^{VS} \left( \boldsymbol{w}\left( 0 \right), \boldsymbol{v}\left( 0 \right) \right), \ \forall t \geq 0. \tag{17}$$

The right-hand-side of (16) grows exponentially with respect to $d$, $k$, and $p$, thereby indicating the substantial influence of these factors on the convergence of VS-PINNs, including PINNs as a specific case ($L = 0$). This analysis reveals several significant advantages of VS-PINNs:

**Improved Convergence:** VS-PINNs are more likely to converge to the global optimum than PINNs due to the reduction in the derivative order $|\xi| < k$. This also relaxes the condition on $p$ from $p \geq k+1$ to $p \geq |\xi| + 1$. As previously discussed in Section 3, the optimal value of $p$ is $|\xi| + 1$. Given that $6\left(|\xi| + 1\right)$ is an exponent of $\log\left(1/\delta\right)$, the most dominant term, reducing the order from $k$ to $|\xi|$ leads to an immense improvement. There is another noteworthy observation we can see here. Given a $k$th-order PDE, there are numerous possible partitions $\xi$ that could be employed to decompose it to a system of lower-order PDEs. Consequently, there are a many of potential VS-PINNs that could be constructed. The aforementioned result indicates which of these is the most effective. As the convergence improves dramatically with a reduction in the derivative order, the optimal approach for splitting variables among various ways is to separate the given PDE into a system of first-order PDEs by parameterizing all derivatives of the solution as auxiliary variables. In other words, the finest splitting with $\xi_0 = 0$, $\xi_1 = 1, \ldots, \xi_{k-1} = k - 1$ would be the most effective in terms of the convergence of GF, as the differential order $|\xi|$ is reduced the most to 1. Taken all together, the most optimal VS-PINNs that reduce the PDE order $k$ to 1 will markedly enhance the convergence of GF.

**Reduced Dimensional Impact:** The reduction of orders in VS-PINNs enhances the resilience of the model to high-dimensionality. From Theorem 3.2, we observed the effect of $d$ being exponentially enlarged for the PDE order $k$ due to the $k$th-order partial differential operators in the loss function. It can be alleviated by VS-PINNs reducing the order, thereby easing the amplified scale to exponential of $|\xi|$. This indicates that VS-PINNs are more effective in combating the curse of dimensionality. Since the curse of dimensionality is a serious issue that is prevalent in various fields, including Hamilton-Jacobi-Bellman equation in control problems, Schrodinger equation in quantum physics, and Black-Scholes equation in finance, it is evident that enhancements to the robustness of VS-PINNs with respect to their dimensionality would facilitate considerable advancements in various fields.

**Memory Efficiency:** VS-PINNs are memory-efficient despite the presence of multiple auxiliary networks. As the order of the derivative increases, the complexity in automatic differentiation in modern deep-learning frameworks like PyTorch increases and it becomes computationally expensive [7]. Adopting the order-reduced representation in the proposed variable splitting can overcome the difficulty in calculating the high-order derivative via automatic differentiation. The loss function for the finest VS-PINNs involves only first-order derivatives, which reduces the memory usage and computational requirements. Despite the increase in the number of networks, VS-PINNs exhibit greater efficiency because memory usage and computation scale linearly with the number of networks in contrast to the exponential scaling with the order of derivatives. Table 3 in appendix demonstrates the memory reduction of VS-PINNs.

*Remark* 4.4. The current approach to parameterizing the $\xi_\ell$-th order differential operator on all axes $\nabla^{\xi_\ell}$ as an auxiliary variable may be suboptimal in certain cases. In a given PDE, if the order of the derivative varies significantly along the axes, that is, $a_\alpha \neq 0$ for only a few $\alpha$ in (1), it may be more efficient to approximate the partial derivatives using auxiliary variables separately for each axis. To illustrate, for the PDE $u_{tt} = u_{xxxx}$, it is more suitable to parameterize variables $\phi_0 \approx u$, $\phi_1 \approx (u_t, u_x)$, $\phi_2 \approx u_{xx}$, and $\phi_3 \approx u_{xxx}$, rather than approximating all tensors $\nabla_{(t,x)}^1$, $\nabla_{(t,x)}^2$, and $\nabla_{(t,x)}^3$. The theoretical framework presented in this paper is capable of addressing this scenario by constructing each $\phi_\ell$ to replace $\frac{\partial^{\xi_\ell}}{\partial \boldsymbol{x}^\alpha}$ for only part of $\alpha$ with $|\alpha| = \xi_\ell$. However, we exclude it due to the intricate nature of the states and the lack of a meaningful impact on the PDE order.

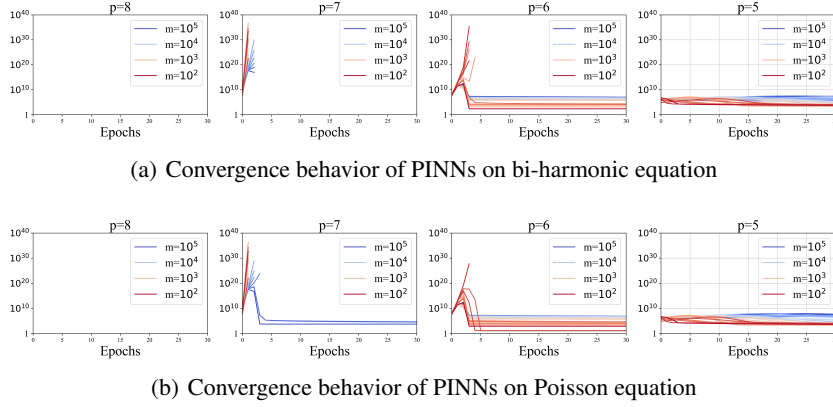

(a) Convergence behavior of PINNs on bi-harmonic equation

(b) Convergence behavior of PINNs on Poisson equation

Figure 1: Training losses of PINNs solving (a) bi-harmonic equation and (b) Poisson equation.

*Remark* 4.5. Although the shaprness of the bound in Theorem 4.3 is open, it is important to note that the leading term of the bound is based on conditions necessary for the Gram matrix to be positive definite, which is a crucial property of the Gram matrix for ensuring the convergence of the GF to a global optimizer. Since the Gram matrix is defined by the PDE loss and the network structure, we believe it can still provide valuable insight into how order and power affect convergence.

# 5 Experiments

This section presents experimental results that validate the theory. Throughout numerical experiments, two-layer MLPs with ReLU$^p$ activation function were utilized in order to align with our theoretical framework. Throughout all experiments, the training collocation points consists of uniform grid and regularization parameters are set to $\nu_1, \ldots, \nu_L = 1$ and $\nu = 10$. We implement all numerical experiments on a single NVIDIA RTX 3090 GPU. Experimental details are provided in Appendix D.

**Convergence behavior of PINNs**    To investigate the influence of the activation order $p$ and the PDE order $k$ on the width $m$ required for convergence, we examined both the second-order Poisson equation and the fourth-order bi-harmonic equation, both of which yield the same solution. We trained networks with varying widths m, ranging from $10^2$ to $10^6$, for each combination of $p$ and $k$ using GD optimization with a learning rate of $10^{-8}$. Figure 1 illustrates the training losses at the initial stage on a logarithmic scale, supporting our theoretical findings that a larger width is needed for higher values of $p$ to ensure convergence. Moreover, we can observe that narrower networks tend to converge more readily when solving lower-order PDEs (Poisson) compared to higher-order PDEs (bi-harmonic). This observation aligns with Theorem 4.3 that higher-order PDEs necessitate larger network widths for guaranteed convergence.

**Validation on the effect of** $p$    To verify the influence of the power $p$ of the ReLU activation function, we test PINNs with varying $p$ values between 3 and 10. Since the training process became highly unstable as $p$ increases, we consider second-order heat equation (509) [14] to gain a more precise investigation of the effect of $p$. The results are summarized in Figure 2 (a). We can see that the convergence of loss is enhanced as $p$ decreases, which supports our theoretical finding.

**Comparison between PINNs and VS-PINNs**    To validate the order reduction effect of VS-PINNs, we conducted an experiment comparing PINNs with VS-PINNs on the second-order heat equation. Each model was run five times with different random seeds, and Figure 2 depicts the training loss for both PINNs and VS-PINNs along with their variance. The results show that the training loss for VS-PINNs converges more effectively than that of PINNs. This indicates that VS-PINNs, which optimize a loss function incorporating lower-order derivatives using networks with smaller $p$, facilitate convergence of GD, consistent with the theoretical findings in Section 4. Furthermore, we performed a similar experiment on the convection-diffusion equation (511) in the Appendix E and obtained results that were consistent with those observed for the heat equation.

**Effect of splitting level**    For higher-order PDEs, there are several ways to transform a given PDE into a lower-order system through variable splitting. To investigate this effect, we conducted experiments on the fourth-order elastic beam equation (510) [52] with two cases: (i) $\phi_0 \approx u, \phi_1 \approx u_t, \phi_2 \approx u_{xx}$

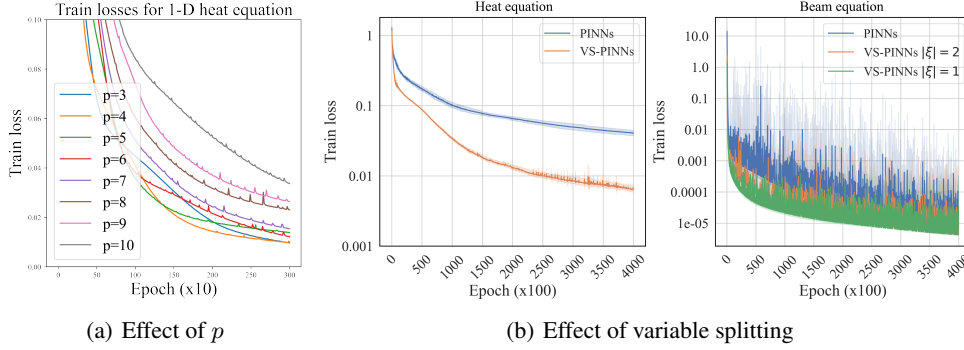

(a) Effect of $p$  ·  (b) Effect of variable splitting

Figure 2: Loss curves of (a) effect of the power $p$ of ReLU$^p$ and (b) comparison between PINNs with VS-PINNs.

with $|\xi| = 2$ and $p = 3$ and (ii) the finest splitting of $\phi_0 \approx u, \phi_1 \approx \nabla u, \phi_2 \approx u_{xx}, \phi_3 \approx u_{xxx}$ with $|\xi| = 1$ and $p = 2$. In order to train PINNs for a fourth-order PDE, $p$ should be at least five, but training such PINNs with GD does not proceed properly, as illustrated in Figure 1. Consequently, the experiments were conducted using the Adam optimizer. In contrast to the underperforming PINNs, VS-PINNs are effectively trained even with GD, as illustrated in Figure 6 of the Appendix E. We run each model five times with different random seeds, and Figure 2 (b) depicts the training loss of PINN and two VS-PINNs with variance. The results show that the model with a lower PDE order $k$ and a smaller power $p$ of the activation exhibits a more pronounced reduction in the loss function, in accordance with our theoretical findings. Furthermore, it can be observed that the variance of the training loss is significantly smaller for the models with smaller values of $k$ and $p$. This indicates that the learning process is much more stable for a smaller $k$ and $p$. We also conduct numerical studies on the fourth-order bi-harmonic equation (508). However, the results exhibit a similar trend to that observed in the beam equation and are therefore presented in Appendix E.

## 6 Conclusion

In this paper, we proved that the gradient flow of PINNs converges to a global minimum and provides sufficient width for this convergence. It extends the results in [25] to general PDEs and activation functions and provides even tighter conditions on the width size. The main theorem demonstrates that the PDE order or dimension exponentially increases the width requirement, theoretically indicating that PINNs are challenging to optimize for high-order or high-dimensional PDEs. We also substantiate that the PDE order amplifies the adverse effects of dimensionality, which explains why PINNs are more susceptible to dimensionality than other deep learning losses without differentiation. Furthermore, We showed that the variable splitting strategy improves convergence by reducing the differential order included in the training loss function.

It is acknowledged that we only provided sufficient conditions for convergence. To fully comprehend the role of these factors in optimizing PINNs, it is also necessary to establish the necessary conditions linking PDE order, dimension, and convergence. Given that the primary goal of PINNs is to approximate the solution of PDEs, it could also be a limitation that all discussions were limited to empirical losses with fixed collocation points. It would therefore be a worthwhile future direction to analyze the conditions under which the expected loss converges when training collocation points are randomly sampled per epoch. Extending our theoretical framework to analyze the impact of the variable splitting strategy on the generalization error of PINNs, as suggested in [64], would also be an interesting and important research direction. Moreover, as our analysis was confined to continuous time flows, a comprehensive understanding of gradient descent would necessitate the analysis of discretized flows, since GF and GD have different dynamics [50]. We expect that our theory could be adapted to GD dynamics by using Theorem 3.3 of [50], which treats GD as GF with a counter term, but we leave it for future work. In a practical context, the convergence of PINNs for adaptive optimizers, such as Adam [35] or L-BFGS [43], and other activation functions, including hyperbolic tangent, remains an open question.

# 7 Acknowledgements

This work was supported by the NRF grant [2021R1A2C3010887, RS-2024-00406127, RS-2024-00343226, RS-2024-00421203] and MSIT/IITP[NO.2021-0-01343Artificial Intelligence Graduate School Program(SNU)].

## Footnotes

*Equal contribution authors. Correspondence to: <mkang@snu.ac.kr>.

[2]The boundary condition for high-order PDEs is typically given by multiple conditions of a higher order than one. Our approach also encompasses such general boundary conditions by incorporating the residuals of each boundary condition into the loss. The sole distinction is the utilization of induction not only on the derivative matching losses but also on boundary losses to prove Proposition 2. For the sake of brevity, we assume that the boundary condition is the most prevalent (weighted combination of) Dirichlet and Neumann conditions.

[3]Indeed, our analysis covers more general initialize distributions than [25]. In our analysis, it is enough that a probability density function of $\boldsymbol{w}\left(0\right)$ is in Schwartz space, and that of $\boldsymbol{v}\left(0\right)$ is bounded and has zero expectation.

[4]In terms of approximating the solution of the PDE, a larger $p$ makes the network smoother and has better expressive power [11]. However, it means that the set of networks covers broader function spaces as $p$ increases,

# References

[1] Ahmed Abdeljawad and Philipp Grohs. Integral representations of shallow neural network with rectified power unit activation function. *Neural Networks*, 155:536–550, 2022.

[2] Laith Alzubaidi, Jinshuai Bai, Aiman Al-Sabaawi, Jose Santamaría, Ahmed Shihab Albahri, Bashar Sami Nayyef Al-dabbagh, Mohammed A Fadhel, Mohamed Manoufali, Jinglan Zhang, Ali H Al-Timemy, et al. A survey on deep learning tools dealing with data scarcity: definitions, challenges, solutions, tips, and applications. *Journal of Big Data*, 10(1):46, 2023.

[3] Abdon Atangana and Ernestine Alabaraoye. Solving a system of fractional partial differential equations arising in the model of hiv infection of cd4+ cells and attractor one-dimensional Keller-Segel equations. *Advances in Difference Equations*, 2013:1–14, 2013.

[4] Roberto Ballarini. The Da Vinci-Euler-Bernoulli Beam theory? *Mechanical Engineering Magazine Online*, 2003.

[5] Guy Barles and Halil Mete Soner. Option pricing with transaction costs and a nonlinear black-scholes equation. *Finance and Stochastics*, 2:369–397, 1998.

[6] Shamsulhaq Basir. Investigating and mitigating failure modes in physics-informed neural networks (PINNs). *Communications in Computational Physics*, 33(5):1240–1269, 2023.

[7] Atilim Gunes Baydin, Barak A Pearlmutter, Alexey Andreyevich Radul, and Jeffrey Mark Siskind. Automatic differentiation in machine learning: a survey. *Journal of machine learning research*, 18(153):1–43, 2018.

[8] Feliks Aleksandrovich Berezin and Mikhail Shubin. *The Schrödinger Equation*, volume 66. Springer Science & Business Media, 2012.

[9] Jiang Bo-Nan and CL Chang. Least-squares finite elements for the stokes problem. *Computer Methods in Applied Mechanics and Engineering*, 78(3):297–311, 1990.

[10] Laxman Bokati, Vladik Kreinovich, Joseph Baca, and Natasha Rovelli. Why rectified power (repu) activation functions are efficient in deep learning: A theoretical explanation. In *Uncertainty, Constraints, and Decision Making*, pages 7–13. Springer, 2023.

[11] Kurt Izak M. Cabanilla, Rhudaina Z. Mohammad, and Jose Ernie C. Lope. Neural networks with relu powers need less depth. *Neural Networks*, 172:106073, 2024.

[12] John W Cahn and John E Hilliard. Free energy of a nonuniform system. i. interfacial free energy. *The Journal of chemical physics*, 28(2):258–267, 1958.

[13] Shengze Cai, Zhiping Mao, Zhicheng Wang, Minglang Yin, and George Em Karniadakis. Physics-informed neural networks (PINNs) for fluid mechanics: A review. *Acta Mechanica Sinica*, 37(12):1727–1738, 2021.

[14] John Rozier Cannon. *The one-dimensional heat equation*. Number 23. Cambridge University Press, 1984.

[15] Charles K Chui, Xin Li, and Hrushikesh Narhar Mhaskar. Neural networks for localized approximation. *mathematics of computation*, 63(208):607–623, 1994.

[16] Bernardo Cockburn and Chi-Wang Shu. The local discontinuous galerkin method for time-dependent convection-diffusion systems. *SIAM journal on numerical analysis*, 35(6):2440–2463, 1998.

[17] Peter Constantin. Euler equations, Navier-Stokes equations and turbulence. In *Mathematical Foundation of Turbulent Viscous Flows: Lectures given at the CIME Summer School held in Martina Franca, Italy, SEptember 1-5, 2003*, pages 1–43. Springer, 2005.

[18] Tim De Ryck, Florent Bonnet, Siddhartha Mishra, and Emmanuel de Bézenac. An operator preconditioning perspective on training in physics-informed machine learning. *arXiv preprint arXiv:2310.05801*, 2023.

[19] Tim De Ryck and Siddhartha Mishra. Error analysis for physics-informed neural networks (pinns) approximating kolmogorov pdes. *Advances in Computational Mathematics*, 48(6):79, 2022.

[20] Lokenath Debnath and Lokenath Debnath. *Nonlinear partial differential equations for scientists and engineers*. Springer, 2005.

[21] MWMG Dissanayake and Nhan Phan-Thien. Neural-network-based approximations for solving partial differential equations. *communications in Numerical Methods in Engineering*, 10(3):195–201, 1994.

[22] E Kirk Donald et al. Optimal control theory: an introduction. *Mineola, NY: Dover Publications, Inc*, 1970.

[23] Nathan Doumèche, Gérard Biau, and Claire Boyer. Convergence and error analysis of pinns. *arXiv preprint arXiv:2305.01240*, 2023.

[24] Simon S Du, Xiyu Zhai, Barnabas Poczos, and Aarti Singh. Gradient descent provably optimizes over-parameterized neural networks. *arXiv preprint arXiv:1810.02054*, 2018.

[25] Yihang Gao, Yiqi Gu, and Michael Ng. Gradient descent finds the global optima of two-layer physics-informed neural networks. In *International Conference on Machine Learning*, pages 10676–10707. PMLR, 2023.

[26] Roland Glowinski, Hao Liu, Shingyu Leung, and Jianliang Qian. A finite element/operator-splitting method for the numerical solution of the two dimensional elliptic monge–ampère equation. *Journal of Scientific Computing*, 79:1–47, 2019.

[27] John B Greer, Andrea L Bertozzi, and Guillermo Sapiro. Fourth order partial differential equations on general geometries. *Journal of Computational Physics*, 216(1):216–246, 2006.

[28] Ehsan Haghighat, Maziar Raissi, Adrian Moure, Hector Gomez, and Ruben Juanes. A physics-informed deep learning framework for inversion and surrogate modeling in solid mechanics. *Computer Methods in Applied Mechanics and Engineering*, 379:113741, 2021.

[29] QiZhi He, David Barajas-Solano, Guzel Tartakovsky, and Alexandre M Tartakovsky. Physics-informed neural networks for multiphysics data assimilation with application to subsurface transport. *Advances in Water Resources*, 141:103610, 2020.

[30] Masanobu Horie and Naoto Mitsume. Physics-embedded neural networks: Graph neural pde solvers with mixed boundary conditions. *Advances in Neural Information Processing Systems*, 35:23218–23229, 2022.

[31] Kurt Hornik, Maxwell Stinchcombe, and Halbert White. Universal approximation of an unknown mapping and its derivatives using multilayer feedforward networks. *Neural networks*, 3(5):551–560, 1990.

[32] K Hosseini, K Sadri, M Mirzazadeh, YM Chu, A Ahmadian, BA Pansera, and S Salahshour. A high-order nonlinear schrödinger equation with the weak non-local nonlinearity and its optical solitons. *Results in Physics*, 23:104035, 2021.

[33] Zheyuan Hu, Zekun Shi, George Em Karniadakis, and Kenji Kawaguchi. Hutchinson trace estimation for high-dimensional and high-order physics-informed neural networks. *Computer Methods in Applied Mechanics and Engineering*, 424:116883, 2024.

[34] Deqing Jiang, Justin Sirignano, and Samuel N Cohen. Global convergence of deep galerkin and pinns methods for solving partial differential equations. *arXiv preprint arXiv:2305.06000*, 2023.

[35] Diederik P. Kingma and Jimmy Ba. Adam: A method for stochastic optimization. *arXiv preprint arXiv:1412.6980*, 2014.

[36] Georgios Kissas, Yibo Yang, Eileen Hwuang, Walter R Witschey, John A Detre, and Paris Perdikaris. Machine learning in cardiovascular flows modeling: Predicting arterial blood pressure from non-invasive 4d flow mri data using physics-informed neural networks. *Computer Methods in Applied Mechanics and Engineering*, 358:112623, 2020.

[37] Aditi Krishnapriyan, Amir Gholami, Shandian Zhe, Robert Kirby, and Michael W Mahoney. Characterizing possible failure modes in physics-informed neural networks. *Advances in Neural Information Processing Systems*, 34:26548–26560, 2021.

[38] Isaac E Lagaris, Aristidis Likas, and Dimitrios I Fotiadis. Artificial neural networks for solving ordinary and partial differential equations. *IEEE transactions on neural networks*, 9(5):987–1000, 1998.

[39] Hyuk Lee and In Seok Kang. Neural algorithm for solving differential equations. *Journal of Computational Physics*, 91(1):110–131, 1990.

[40] Qianxiao Li, Cheng Tai, and E Weinan. Stochastic modified equations and dynamics of stochastic gradient algorithms i: Mathematical foundations. *Journal of Machine Learning Research*, 20(40):1–47, 2019.

[41] Xin Li. Simultaneous approximations of multivariate functions and their derivatives by neural networks with one hidden layer. *Neurocomputing*, 12(4):327–343, 1996.

[42] Zongyi Li, Nikola Kovachki, Kamyar Azizzadenesheli, Burigede Liu, Kaushik Bhattacharya, Andrew Stuart, and Anima Anandkumar. Fourier neural operator for parametric partial differential equations. *arXiv preprint arXiv:2010.08895*, 2020.

[43] Dong C Liu and Jorge Nocedal. On the limited memory bfgs method for large scale optimization. *Mathematical programming*, 45(1):503–528, 1989.

[44] Minliang Liu, Liang Liang, and Wei Sun. A generic physics-informed neural network-based constitutive model for soft biological tissues. *Computer methods in applied mechanics and engineering*, 372:113402, 2020.

[45] Yu Liu and Wentao Ma. Gradient auxiliary physics-informed neural network for nonlinear biharmonic equation. *Engineering Analysis with Boundary Elements*, 157:272–282, 2023.

[46] El Mehdi Lotfi, Mehdi Maziane, Khalid Hattaf, and Noura Yousfi. Partial differential equations of an epidemic model with spatial diffusion. *International Journal of Partial Differential Equations*, 2014:1–6, 2014.

[47] Tao Luo and Haizhao Yang. Two-layer neural networks for partial differential equations: Optimization and generalization theory. *arXiv preprint arXiv:2006.15733*, 2020.

[48] Revanth Mattey and Susanta Ghosh. A physics informed neural network for time-dependent nonlinear and higher order partial differential equations. *arXiv preprint arXiv:2106.07606*, 2021.

[49] Siddhartha Mishra and Roberto Molinaro. Estimates on the generalization error of physics-informed neural networks for approximating a class of inverse problems for pdes. *IMA Journal of Numerical Analysis*, 42(2):981–1022, 2022.

[50] Taiki Miyagawa. Toward equation of motion for deep neural networks: Continuous-time gradient descent and discretization error analysis. *Advances in Neural Information Processing Systems*, 35:37778–37791, 2022.

[51] Eike H Müller and Robert Scheichl. Massively parallel solvers for elliptic partial differential equations in numerical weather and climate prediction. *Quarterly Journal of the Royal Meteorological Society*, 140(685):2608–2624, 2014.

[52] Frithiof I Niordson. On the optimal design of a vibrating beam. *Quarterly of Applied Mathematics*, 23(1):47–53, 1965.

[53] Bernt Oksendal. *Stochastic differential equations: an introduction with applications*. Springer Science & Business Media, 2013.

[54] Yesom Park, Taekyung Lee, Jooyoung Hahn, and Myungjoo Kang. $p$-Poisson surface reconstruction in curl-free flow from point clouds. *Advances in Neural Information Processing Systems*, 2023.

[55] Yesom Park, Chang Hoon Song, Jooyoung Hahn, and Myungjoo Kang. ReSDF: Redistancing implicit surfaces using neural networks. *Journal of Computational Physics*, 502, 2024.

[56] Yesom Park, Changhoon Song, and Myungjoo Kang. Beyond derivative pathology of pinns: Variable splitting strategy with convergence analysis. *arXiv preprint arXiv:2409.20383*, 2024.

[57] Shige Peng. A generalized dynamic programming principle and hamilton-jacobi-bellman equation. *Stochastics: An International Journal of Probability and Stochastic Processes*, 38(2):119–134, 1992.

[58] Maziar Raissi, Paris Perdikaris, and George E Karniadakis. Physics-informed neural networks: A deep learning framework for solving forward and inverse problems involving nonlinear partial differential equations. *Journal of Computational physics*, 378:686–707, 2019.

[59] Justin Sirignano and Konstantinos Spiliopoulos. DGM: A deep learning algorithm for solving partial differential equations. *Journal of computational physics*, 375:1339–1364, 2018.

[60] Roger Temam. *Navier-Stokes equations: theory and numerical analysis*, volume 343. American Mathematical Soc., 2001.

[61] Sifan Wang, Yujun Teng, and Paris Perdikaris. Understanding and mitigating gradient flow pathologies in physics-informed neural networks. *SIAM Journal on Scientific Computing*, 43(5):A3055–A3081, 2021.

[62] Sifan Wang, Xinling Yu, and Paris Perdikaris. When and why pinns fail to train: A neural tangent kernel perspective. *Journal of Computational Physics*, 449:110768, 2022.

[63] Jue Yan and Chi-Wang Shu. Local discontinuous galerkin methods for partial differential equations with higher order derivatives. *Journal of Scientific Computing*, 17:27–47, 2002.

[64] Marius Zeinhofer, Rami Masri, and Kent-André Mardal. A unified framework for the error analysis of physics-informed neural networks. *arXiv preprint arXiv:2311.00529*, 2023.

[65] Enrui Zhang, Ming Dao, George Em Karniadakis, and Subra Suresh. Analyses of internal structures and defects in materials using physics-informed neural networks. *Science advances*, 8(7):eabk0644, 2022.

# A Mathematical Notations

We introduce the symbols and mathematical notations that are frequently used in this paper.

| Symbol | Description |
|---|---|
| $\mathbb{N}_0$ | The set of non-negative integers |
| $d \in \mathbb{N}$ | dimension of domain |
| $\Omega \subset \mathbb{R}^d$ | $d$-dimensional domain |
| $\alpha, \beta \in \mathbb{N}_0^d$ | multi-index of dimension $d$ |
| $k \in \mathbb{N}$ | order of the governing PDE |
| $\mathcal{N}$ | differential operator |
| $\hat{\mathcal{N}}$ | Splitted differential operator |
| $\mathcal{B}$ | boundary condition operator |
| $\xi$ | integral partition of $[0, k]$ |
| $f : \Omega \to \mathbb{R}$ | source function |
| $g : \partial\Omega \to \mathbb{R}$ | boundary function |
| $\boldsymbol{x}\,(\boldsymbol{y})$ | (augmented)point in $\Omega$ |
| $\tilde{\boldsymbol{x}}\,(\tilde{\boldsymbol{y}})$ | (augmented)point on $\partial\Omega$ |
| $\phi$ | neural network |
| $\boldsymbol{G_w}, \boldsymbol{G_v}$ | Gram matrices |
| $\lambda_0$ | The minimal eigenvalue of $\boldsymbol{G_v}$ |
| $m \in \mathbb{N}$ | the number of the width of the network |
| $\sigma$ | activation function |
| $p \in \mathbb{N}$ | power of ReLU activation function |
| $n_o, n_b \in \mathbb{N}$ | the number of collocation points sampled from the domain $\Omega$, $\partial\Omega$ |
| $\boldsymbol{s}$ | residual of PDE loss |
| $\boldsymbol{\mu}$ | residual of gradient matching loss |
| $\boldsymbol{h}$ | residual of boundary loss |

For a positive integer $m$, the set $\{1, \cdots, m\}$ is abbreviated as $[m]$. The set of multi-indexes whose size is $m$ or is at most $m$ is referred by

$$
\begin{aligned}
I_m &= \left\{ \alpha = (\alpha_1, \ldots, \alpha_d) \in \mathbb{N}_0^d : |\alpha| = \sum_{i=1}^d \alpha_i = m \right\}, \\
J_m &= \left\{ \alpha = (\alpha_1, \ldots, \alpha_d) \in \mathbb{N}_0^d : |\alpha| = \sum_{i=1}^d \alpha_i \le m \right\}.
\end{aligned}
\tag{18}
$$

In this paper, we use two-layer MLPs. For given $k \in \mathbb{N}$ and a partition $0 = \xi_0 < \xi_1 < \cdots < \xi_L < \xi_{L+1} = k$, we define $\phi_\ell : \mathbb{R}^d \to \mathbb{R}^{|I_{\xi_\ell}|}$ of width $m$ as

$$
\begin{aligned}
\phi_\ell\left(\boldsymbol{x}; \boldsymbol{w}_\ell, \boldsymbol{v}_\ell\right) &= \frac{1}{\sqrt{m}} \sum_{r=1}^m (\boldsymbol{v}_\ell)_r\, \sigma\left( \begin{bmatrix} (\boldsymbol{w}_\ell)_{r,1} & (\boldsymbol{w}_\ell)_{r,2} & \cdots & (\boldsymbol{w}_\ell)_{r,d} \end{bmatrix} \boldsymbol{x} + \frac{1}{2}(\boldsymbol{w}_\ell)_{r,d+1} \right) \\
&= \frac{1}{\sqrt{m}} \sum_{r=1}^m (\boldsymbol{v}_\ell)_r\, \sigma\left( (\boldsymbol{w}_\ell)_r^\top\, \boldsymbol{y} \right),
\end{aligned}
\tag{19}
$$

where weights in the first and second layer are

$$
(\boldsymbol{w}_\ell)_r = \begin{bmatrix} (\boldsymbol{w}_\ell)_{r,1} & (\boldsymbol{w}_\ell)_{r,2} & \cdots & (\boldsymbol{w}_\ell)_{r,d} & \frac{1}{2}(\boldsymbol{w}_\ell)_{r,d+1} \end{bmatrix}^\top \in \mathbb{R}^{(d+1)},
\tag{20}
$$

$$
(\boldsymbol{v}_\ell)_r = \begin{bmatrix} (\boldsymbol{v}_\ell)_{r,1} & (\boldsymbol{v}_\ell)_{r,2} & \cdots & (\boldsymbol{v}_\ell)_{r,|I_{\xi_\ell}|} \end{bmatrix}^\top \in \mathbb{R}^{|I_{\xi_\ell}|},
\tag{21}
$$

and $\boldsymbol{y} = \begin{bmatrix} \boldsymbol{x}^\top & \frac{1}{2} \end{bmatrix}^\top \in \mathbb{R}^{d+1}$. Note that the output dimension of $\phi_\ell$ is $\mathbb{R}^{|I_{\xi_\ell}|}$ because each component of $\phi_\ell\left(\boldsymbol{x}; \boldsymbol{w}_\ell, \boldsymbol{v}_\ell\right)$ represents partial derivative $\frac{\partial^{\xi_\ell}}{\partial \boldsymbol{x}^\alpha} u$, for each $\alpha \in I_{\xi_\ell}$. Hence, even though $(\boldsymbol{v}_\ell)_r$ is a flattened vector, it is more convenient to use $\alpha \in I_{\xi_\ell}$ as an index of component of $(\boldsymbol{v}_\ell)_r$. For example, if $d = 2$, $\xi_1 = 1$, and $\alpha = (0, 1) \in I_1$, $(\boldsymbol{v}_1)_{r,\alpha} = (\boldsymbol{v}_1)_{r,(0,1)}$ refers a weight between the

$r$-th hidden node and output that represents $\frac{\partial}{\partial y} u(x, y)$. Similarly, we use multi-index itself as an index that is related to $I_{\xi_\ell}$, $I_{\Delta \xi_\ell}$, or $J_{\Delta \xi_\ell}$.

Moreover, as each $(\phi_\ell)_\alpha$ is differentiated by some $\frac{\partial^{\Delta \xi_\ell}}{\partial \boldsymbol{x}^\beta}$ for $\alpha \in I_{\xi_{\ell-1}}$ and $\beta \in I_{\Delta \xi_\ell}$, index we use pair of multi-index $(\alpha, \beta)$ to refer a component of $\frac{\partial^{\Delta \xi_\ell}}{\partial \boldsymbol{x}^\beta}(\phi_\ell)_{r, \alpha}$. Sometimes, this indexing by multi-index is used in conjunction with normal indexes, like $(\boldsymbol{\mu}_\ell)_{(\alpha, \beta), i}$ for some $i \in [n_o]$.

We denote the collection of weights of all $\phi_\ell$'s by $\boldsymbol{w} = \left[\boldsymbol{w}_0^\top, \cdots \boldsymbol{w}_L^\top\right]^\top$ and $\boldsymbol{v} = \left[\boldsymbol{v}_0^\top, \cdots \boldsymbol{v}_L^\top\right]^\top$. Similarly, $\boldsymbol{\mu}$ is the collection of all $\boldsymbol{\mu}_\ell$'s, $\boldsymbol{\mu} = \left[\boldsymbol{\mu}_0^\top, \cdots, \boldsymbol{\mu}_L^\top\right]$. With regularization parameters $\nu, \nu_\ell > 0$, residuals $s_i$, $(\boldsymbol{\mu}_\ell) i$, and $h_j$ of variable splitting for each training sample

$$s_i(\boldsymbol{w}, \boldsymbol{v}) = \sqrt{\frac{1}{n_o}} \left(\hat{\mathcal{N}}\left[\phi_0(\cdot; \boldsymbol{w}_0, \boldsymbol{v}_0), \cdots, \phi_L(\cdot; \boldsymbol{w}_L, \boldsymbol{v}_L)\right](\boldsymbol{x}_i) - f(\boldsymbol{x}_i)\right) \tag{22}$$

$$(\boldsymbol{\mu}_\ell)_i(\boldsymbol{w}, \boldsymbol{v}) = \left[(\boldsymbol{\mu}_\ell)_{(\alpha, \beta), i}(\boldsymbol{w}, \boldsymbol{v})\right]_{\alpha \in I_{\xi_{\ell-1}}, \beta \in I_{\Delta \xi_\ell}} \tag{23}$$

$$= \left[\frac{\partial^\beta}{\partial \boldsymbol{x}^\beta}(\phi_{\ell-1})_\alpha(\boldsymbol{x}_i; \boldsymbol{w}_{\ell-1}, \boldsymbol{v}_{\ell-1}) - \phi_\ell(\boldsymbol{x}_i; \boldsymbol{w}_\ell, \boldsymbol{v}_\ell)_{\alpha+\beta}\right]_{\alpha \in I_{\xi_{\ell-1}}, \beta \in I_{\Delta \xi_\ell}}, \tag{24}$$

and

$$h_j(\boldsymbol{w}, \boldsymbol{v}) = \sqrt{\frac{\nu}{n_b}} \left(\mathcal{B}\left[\phi(\cdot; \boldsymbol{w}_0, \boldsymbol{v}_0)\right](\tilde{\boldsymbol{x}}_j) - g(\tilde{\boldsymbol{x}}_j)\right) \tag{25}$$

define the loss $\mathcal{L}_{PINN}^{VS}$ for the system,

$$\mathcal{L}_{PINN}^{VS}(\boldsymbol{w}, \boldsymbol{v}) \tag{26}$$

$$= \frac{1}{2}\left(\sum_{i=1}^{n_o} s_i(\boldsymbol{w}, \boldsymbol{v})^2 + \sum_{i=1}^{n_o}\sum_{\ell=1}^{L}\sum_{\alpha \in I_{\xi_{\ell-1}}}\sum_{\beta \in J_{\Delta \xi_\ell}}(\boldsymbol{\mu}_\ell)_{(\alpha, \beta), i}(\boldsymbol{w}, \boldsymbol{v})^2 + \sum_{j=1}^{n_b} h_j(\boldsymbol{w}, \boldsymbol{v})^2\right). \tag{27}$$

we define

$$a_{\max} = \max\left(\left\{|\hat{a}_{\ell, \alpha, \beta}(\boldsymbol{x}_i)| : \begin{array}{l} 0 \le \ell \le L, i \in [n_o], \\ \alpha \in I_{\xi_\ell}, \beta \in J_{\Delta \xi_\ell} \end{array}\right\} \cup \left\{|\tilde{a}_\alpha(\tilde{\boldsymbol{x}}_j)| : \begin{array}{l} \alpha \in J_1, \\ j \in [n_b] \end{array}\right\}\right), \tag{28}$$

$$\nu_{\max} = \max\left(\{1, \nu\} \cup \{\nu_\ell : \ell \in [L]\}\right). \tag{29}$$

## B  Calculations

The proof of the main theorem includes intricate calculations. To keep the proof clear, we separate some tedious computations that are used frequently. As this section is a reference for the proofs, we use some notation or symbols without any mention if they are defined in the other part of the paper.

**Sizes of index set $I_m$ and $J_m$.**  For any $\ell \in [L]$,

$$|I_{\xi_\ell}| = \binom{d + \xi_\ell - 1}{\xi_\ell}, |I_{\Delta \xi_\ell}| = \binom{d + \Delta \xi_\ell - 1}{\Delta \xi_\ell}, |J_{\Delta \xi_\ell}| = \binom{d + \Delta \xi_\ell}{\Delta \xi_\ell}. \tag{30}$$

We bound $|I_{\xi_\ell}|$, $|\Delta \xi_\ell|$ and $J_{\Delta \xi_\ell}$ by

$$|I_{\Delta \xi_\ell}| \le |J_{\Delta \xi_\ell}| \le \binom{d + |\xi|}{d}, \tag{31}$$

and

$$\sum_{\ell=0}^{L}|I_{\xi_\ell}| \le |J_{\xi_L}| \le |J_{k-1}| = \binom{d + k}{d}. \tag{32}$$

**Partial derivative of $\hat{s}$, $\mu$, and $h$.** Note that $\frac{\partial^\beta}{\partial x^\beta}\left(\left(w_\ell\right)_r^\top y_i\right)_+^p$ in (34), $\frac{\partial^\beta}{\partial x^\beta}\left(\left(w_{\ell-1}\right)_r^\top y_i\right)_+^p$ in (35), $\left(\left(w_\ell\right)_r^\top y_i\right)_+^p$ in (36), and $\frac{\partial^\alpha}{\partial x^\alpha}\left(\left(w_0\right)_r^\top \tilde{y}_j\right)_+^p$ in (37) are polynomial of $\left(w_\ell\right)_r$, $\left(w_{\ell-1}\right)_r$, $\left(w_\ell\right)_r$, and $\left(w_0\right)_r$ of degree $p$, respectively.

For $i \in [n_o]$, $\ell = 0, 1, \ldots, L$, $r \in [m]$, and $\alpha \in I_{\xi_\ell}$,

$$\frac{\partial}{\partial \left(v_\ell\right)_{r,\alpha}} s_i = \sqrt{\frac{1}{n_o}} \sum_{\beta \in J_{\Delta\xi_{\ell+1}}} \hat{a}_{\ell,\alpha,\beta} \frac{\partial^\beta}{\partial x^\beta} \frac{\partial}{\partial \left(v_\ell\right)_{r,\alpha}} \phi_\ell\left(x_i\right) \tag{33}$$

$$= \sqrt{\frac{1}{mn_o}} \sum_{\beta \in J_{\Delta\xi_{\ell+1}}} \hat{a}_{\ell,\alpha,\beta} \frac{\partial^\beta}{\partial x^\beta} \left(\left(w_\ell\right)_r^\top y_i\right)_+^p. \tag{34}$$

Similarly, for $\ell \in [L]$, $r \in [m]$, $\alpha_1 \in I_{\xi_{\ell-1}}$, $\alpha_2 \in I_{\xi_{\ell-1}}$, $\beta \in I_{\Delta\xi_\ell}$, and $i \in [n_o]$, we have

$$\frac{\partial}{\partial \left(v_{\ell-1}\right)_{r,\alpha_1}} \left(\mu_\ell\right)_{(\alpha_2,\beta),i} = \begin{cases} \sqrt{\frac{\nu_\ell}{mn_o}}\left(\frac{\partial^\beta}{\partial x^\beta}\left(\left(w_{\ell-1}\right)_r^\top y_i\right)_+^p\right) & \text{if } \alpha_1 = \alpha_2 \\ 0 & \text{otherwise.} \end{cases} \tag{35}$$

For $\ell \in [L]$, $r \in [m]$, $\alpha_1 \in I_{\xi_\ell}$, $\alpha_2 \in I_{\xi_{\ell-1}}$, $\beta \in I_{\Delta\xi_\ell}$, and $i \in [n_o]$,

$$\frac{\partial}{\partial \left(v_\ell\right)_{r,\alpha_1}} \left(\mu_\ell\right)_{(\alpha_2,\beta),i} = \begin{cases} -\sqrt{\frac{\nu_\ell}{mn_o}}\left(\left(w_\ell\right)_r^\top y_i\right)_+^p & \text{if } \alpha_1 = \alpha_2 + \beta, \\ 0 & \text{otherwise.} \end{cases} \tag{36}$$

For $r \in [m]$ and $j \in [n_b]$,

$$\frac{\partial}{\partial \left(v_0\right)_{r,1}} h_j = \sqrt{\frac{\nu}{mn_b}} \sum_{\alpha \in J_1} \tilde{a}_\alpha \frac{\partial^\alpha}{\partial x^\alpha} \left(\left(w_0\right)_r^\top \tilde{y}_j\right)_+^p. \tag{37}$$

**Norms of partial derivative of $\hat{s}$, $\mu$, and $h$.** Note that for $i \in [n_o]$, $\ell = 0, 1, \ldots, L$, $r \in [m]$,

$$\left\|\frac{\partial s_i\left(w,v\right)}{\partial \left(w_\ell\right)_r}\right\|_2 = \sqrt{\frac{1}{mn_o}}\left\|\frac{\partial}{\partial \left(w_\ell\right)_r}\left(\sum_{\alpha \in I_{\xi_\ell}} \sum_{\beta \in J_{\Delta\xi_{\ell+1}}} \hat{a}_{\ell,\alpha,\beta} \frac{\partial^\beta}{\partial x^\beta}\left(\left(v_\ell\right)_r\left(\left(w_\ell\right)_r^\top y\right)_+^p\right)_\alpha\right)\right\|_2 \tag{38}$$

$$= \sqrt{\frac{1}{mn_o}}\left\|\frac{\partial}{\partial \left(w_\ell\right)_r}\left(\sum_{\alpha \in I_{\xi_\ell}} \sum_{\beta \in J_{\Delta\xi_{\ell+1}}} \hat{a}_{\ell,\alpha,\beta} \frac{\partial^\beta}{\partial x^\beta}\left(\left(v_\ell\right)_{r,\alpha}\left(\left(w_\ell\right)_r^\top y\right)_+^p\right)\right)\right\|_2 \tag{39}$$

$$= \sqrt{\frac{1}{mn_o}}\left\|\left(\sum_{\alpha \in I_{\xi_\ell}} \left(v_\ell\right)_{r,\alpha} \sum_{\beta \in J_{\Delta\xi_{\ell+1}}} \hat{a}_{\ell,\alpha,\beta} \frac{\partial}{\partial \left(w_\ell\right)_r}\left(\frac{\partial^\beta}{\partial x^\beta}\left(\left(w_\ell\right)_r^\top y\right)_+^p\right)\right)\right\|_2 \tag{40}$$

$$\le \sqrt{\frac{1}{mn_o}} a_{\max} \sum_{\alpha \in I_{\xi_\ell}} \left|\left(v_\ell\right)_{r,\alpha}\right| \sum_{\beta \in J_{\Delta\xi_{\ell+1}}} \left\|\frac{\partial}{\partial \left(w_\ell\right)_r}\left(\frac{\partial^\beta}{\partial x^\beta}\left(\left(w_\ell\right)_r^\top y\right)_+^p\right)\right\|_2 \tag{41}$$

$$\le \sqrt{\frac{1}{mn_o}} a_{\max} \sum_{\alpha \in I_{\xi_\ell}} \left|\left(v_\ell\right)_{r,\alpha}\right| \sum_{\beta \in J_{\Delta\xi_{\ell+1}}} p^{\Delta\xi_{\ell+1}+1} \left\|\left(w_\ell\right)_r\right\|_2^{p-1} \tag{42}$$

$$= \sqrt{\frac{1}{mn_o}} a_{\max} \sum_{\alpha \in I_{\xi_\ell}} \left|\left(v_\ell\right)_{r,\alpha}\right| \left|J_{\Delta\xi_{\ell+1}}\right| p^{\Delta\xi_{\ell+1}+1} \left\|\left(w_\ell\right)_r\right\|_2^{p-1} \tag{43}$$

$$\le \sqrt{\frac{1}{mn_o}} a_{\max} \sqrt{\left|I_{\xi_\ell}\right|} \left\|\left(v_\ell\right)_r\right\|_2 \left|J_{\Delta\xi_{\ell+1}}\right| p^{\Delta\xi_{\ell+1}+1} \left\|\left(w_\ell\right)_r\right\|_2^{p-1}, \tag{44}$$

for $\ell \in [L]$, $r \in [m]$, $\alpha \in I_{\xi_{\ell-1}}$, $\beta \in I_{\Delta\xi_\ell}$, and $i \in [n_o]$,

$$\left\| \frac{\partial \left( \boldsymbol{\mu}_\ell \right)_{(\alpha,\beta),i} (\boldsymbol{w}, \boldsymbol{v})}{\partial \left( \boldsymbol{w}_{\ell-1} \right)_r} \right\|_2 = \left\| \frac{\partial}{\partial \left( \boldsymbol{w}_{\ell-1} \right)_r} \left( \sqrt{\frac{\nu_\ell}{mn_o}} \frac{\partial^\beta}{\partial \boldsymbol{x}^\beta} \left( (\boldsymbol{v}_{\ell-1})_{r,\alpha} \left( (\boldsymbol{w}_{\ell-1})_r^\top \boldsymbol{y}_i \right)_+^p \right) \right) \right\|_2 \tag{45}$$

$$= \sqrt{\frac{\nu_\ell}{mn_o}} \left| (\boldsymbol{v}_{\ell-1})_{r,\alpha} \right| \left\| \frac{\partial}{\partial \left( \boldsymbol{w}_{\ell-1} \right)_r} \left( \frac{\partial^\beta}{\partial \boldsymbol{x}^\beta} \left( \left( (\boldsymbol{w}_{\ell-1})_r^\top \boldsymbol{y}_i \right)_+^p \right) \right) \right\|_2 \tag{46}$$

$$\leq \sqrt{\frac{\nu_\ell}{mn_o}} \left| (\boldsymbol{v}_{\ell-1})_{r,\alpha} \right| p^{\Delta\xi_\ell+1} \| (\boldsymbol{w}_{\ell-1})_r \|_2^{p-1} \tag{47}$$

$$\leq \sqrt{\frac{\nu_{\max}}{mn_o}} p^{|\xi|+1} \| (\boldsymbol{v}_{\ell-1})_r \|_2 \| (\boldsymbol{w}_{\ell-1})_r \|_2^{p-1}, \tag{48}$$

$$\left\| \frac{\partial \left( \boldsymbol{\mu}_\ell \right)_{(\alpha,\beta),i} (\boldsymbol{w}, \boldsymbol{v})}{\partial \left( \boldsymbol{w}_\ell \right)_r} \right\|_2 = \left\| \frac{\partial}{\partial \left( \boldsymbol{w}_\ell \right)_r} \left( \sqrt{\frac{\nu_\ell}{mn_o}} \left( (\boldsymbol{v}_\ell)_{r,\alpha} \left( (\boldsymbol{w}_\ell)_r^\top \boldsymbol{y}_i \right)_+^p \right) \right) \right\|_2 \tag{49}$$

$$= \sqrt{\frac{\nu_\ell}{mn_o}} \left\| (\boldsymbol{v}_\ell)_{r,\alpha} \frac{\partial}{\partial \left( \boldsymbol{w}_\ell \right)_r} \left( (\boldsymbol{w}_\ell)_r^\top \boldsymbol{y}_i \right)_+^p \right\|_2 \tag{50}$$

$$\leq \sqrt{\frac{\nu_\ell}{mn_o}} p \left| (\boldsymbol{v}_\ell)_{r,\alpha} \right| \| (\boldsymbol{w}_\ell)_r \|_2^{p-1} \tag{51}$$

$$\leq \sqrt{\frac{\nu_{\max}}{mn_o}} p \| (\boldsymbol{v}_\ell)_r \|_2 \| (\boldsymbol{w}_\ell)_r \|_2^{p-1}, \tag{52}$$

$$\left\| \frac{\partial \left( \boldsymbol{\mu}_\ell \right)_i (\boldsymbol{w}, \boldsymbol{v})}{\partial \left( \boldsymbol{w}_{\ell'} \right)_r} \right\|_2 = 0, \text{ if } \ell' \neq \ell-1, \ell, \tag{53}$$

and for $r \in [m]$ and $j \in [n_b]$,

$$\left\| \frac{\partial h_j (\boldsymbol{w}, \boldsymbol{v})}{\partial \left( \boldsymbol{w}_0 \right)_r} \right\|_2 = \left\| \frac{\partial}{\partial \left( \boldsymbol{w}_0 \right)_r} \left( \sqrt{\frac{\nu}{mn_b}} \sum_{\alpha \in J_1} \tilde{a}_\alpha \frac{\partial^\alpha}{\partial \boldsymbol{x}^\alpha} \left( (v_0)_{r,\boldsymbol{0}} \cdot \left( (\boldsymbol{w}_0)_r^\top \tilde{\boldsymbol{y}}_j \right)_+^p \right) \right) \right\|_2 \tag{54}$$

$$\leq \frac{\sqrt{\nu}}{\sqrt{mn_b}} a_{\max} \sum_{\alpha \in J_1} \left\| \frac{\partial}{\partial \left( \boldsymbol{w}_0 \right)_r} \frac{\partial^\alpha}{\partial \boldsymbol{x}^\alpha} \left( (v_0)_{r,\boldsymbol{0}} \cdot \left( (\boldsymbol{w}_0)_r^\top \tilde{\boldsymbol{y}}_j \right)_+^p \right) \right\|_2 \tag{55}$$

$$\leq \frac{\sqrt{\nu}}{\sqrt{mn_b}} a_{\max} \sum_{\alpha \in J_1} p^2 \left| (\boldsymbol{v}_0)_{r,\boldsymbol{0}} \right| \| (\boldsymbol{w}_0)_r \|_2^{p-1} \tag{56}$$

$$\leq \sqrt{\frac{\nu_{\max}}{mn_b}} a_{\max} (d+1) p^2 \| (\boldsymbol{v}_0)_r \|_2 \| (\boldsymbol{w}_0)_r \|_2^{p-1}, \tag{57}$$

$$\left\| \frac{\partial h_j (\boldsymbol{w}, \boldsymbol{v})}{\partial \left( \boldsymbol{w}_{\ell'} \right)_{r,1}} \right\|_2 = 0 \text{ if } \ell' \in [L]. \tag{58}$$

Similarly, for $\ell \in [L]$, $r \in [m]$, and $i \in [n_o]$, we have the following inequalities:
for $\alpha \in I_{\xi_\ell}$,

$$\left| \frac{\partial s_i (\boldsymbol{w}, \boldsymbol{v})}{\partial \left( \boldsymbol{v}_\ell \right)_{r,\alpha}} \right| \leq \sqrt{\frac{1}{mn_o}} \sum_{\beta \in J_{\Delta\xi_{\ell+1}}} \left| \hat{a}_{\ell,\alpha,\beta} \frac{\partial^\beta}{\partial \boldsymbol{x}^\beta} \left( (\boldsymbol{w}_\ell)_r^\top \boldsymbol{y}_i \right)_+^p \right| \tag{59}$$

$$\leq \sqrt{\frac{1}{mn_o}} a_{\max} \left| J_{\Delta\xi_{\ell+1}} \right| p^{\Delta\xi_{\ell+1}} \| (\boldsymbol{w}_\ell)_r \|_2^p, \tag{60}$$

$$\left\| \frac{\partial s_i (\boldsymbol{w}, \boldsymbol{v})}{\partial \left( \boldsymbol{v}_\ell \right)_r} \right\|_2 \leq \sqrt{\frac{\nu_{\max}}{mn_o}} a_{\max} \sqrt{\left| I_{\xi_\ell} \right|} \left| J_{\Delta\xi_{\ell+1}} \right| p^{|\xi|} \| (\boldsymbol{w}_\ell)_r \|_2^p \tag{61}$$

for $\alpha_1 \in I_{\xi_{\ell-1}}$, $\alpha_2 \in I_{\xi_{\ell-1}}$, and $\beta \in I_{\Delta\xi_\ell}$,

$$\left| \frac{\partial \left( \boldsymbol{\mu}_\ell \right)_{(\alpha_2,\beta),i} (\boldsymbol{w}(t), \boldsymbol{v}(t))}{\partial \left( \boldsymbol{v}_{\ell-1} \right)_{r,\alpha_1}} \right| = \mathbf{1}_{\{\alpha_1 = \alpha_2\}} \cdot \sqrt{\frac{\nu_\ell}{mn_o}} \left| \frac{\partial^\beta}{\partial \boldsymbol{x}^\beta} \left( (\boldsymbol{w}_{\ell-1})_r^\top \boldsymbol{y}_i \right)_+^p \right| \tag{62}$$

$$\leq \mathbf{1}_{\{\alpha_1 = \alpha_2\}} \cdot \sqrt{\frac{\nu_\ell}{mn_o}} p^{\Delta\xi_\ell} \left\| (\boldsymbol{w}_{\ell-1})_r \right\|_2^p \tag{63}$$

$$\left\| \frac{\partial (\boldsymbol{\mu}_\ell)_{(\alpha_2,\beta),i} (\boldsymbol{w}, \boldsymbol{v})}{\partial (\boldsymbol{v}_{\ell-1})_r} \right\|_2 \leq \sqrt{\frac{\nu_\ell}{mn_o}} p^{\Delta\xi_\ell} \sqrt{\left| I_{\xi_{\ell-1}} \right| \left| I_{\Delta\xi_\ell} \right|} \left\| (\boldsymbol{w}_{\ell-1})_r \right\|_2^p, \tag{64}$$

for $\alpha_1 \in I_{\xi_\ell}$, $\alpha_2 \in I_{\xi_{\ell-1}}$, and $\beta \in I_{\Delta\xi_\ell}$,

$$\left| \frac{\partial (\boldsymbol{\mu}_\ell)_{(\alpha_2,\beta),i} (\boldsymbol{w}, \boldsymbol{v})}{\partial (\boldsymbol{v}_\ell)_{r,\alpha_1}} \right| \leq \mathbf{1}_{\{\alpha_1 = \alpha_2 + \beta\}} \cdot \sqrt{\frac{\nu_\ell}{mn_o}} \left( (\boldsymbol{w}_\ell)_r^\top \boldsymbol{y}_i \right)_+^p \tag{65}$$

$$\leq \mathbf{1}_{\{\alpha_1 = \alpha_2 + \beta\}} \cdot \sqrt{\frac{\nu_\ell}{mn_o}} \left\| (\boldsymbol{w}_\ell)_r \right\|_2^p \tag{66}$$

$$\left\| \frac{\partial (\boldsymbol{\mu}_\ell)_{(\alpha_2,\beta),i} (\boldsymbol{w}, \boldsymbol{v})}{\partial (\boldsymbol{v}_\ell)_r} \right\|_2 \leq \sqrt{\frac{\nu_\ell}{mn_o}} \sqrt{\left| I_{\xi_{\ell-1}} \right| \left| I_{\Delta\xi_\ell} \right|} \left\| (\boldsymbol{w}_\ell)_r \right\|_2^p, \tag{67}$$

for $\ell' \neq \ell - 1, \ell$,

$$\left\| \frac{\partial (\boldsymbol{\mu}_\ell)_i (\boldsymbol{w}, \boldsymbol{v})}{\partial (\boldsymbol{v}_{\ell'})_r} \right\|_2 = 0, \tag{68}$$

and

$$\left| \frac{\partial h_j (\boldsymbol{w}, \boldsymbol{v})}{\partial (\boldsymbol{v}_0)_{r,\mathbf{0}}} \right| \leq \sqrt{\frac{\nu}{mn_b}} \sum_{\alpha \in J_1} \left| \tilde{a}_\alpha \frac{\partial^\alpha}{\partial \boldsymbol{x}^\alpha} \left( \left( (\boldsymbol{w}_0)_r^\top \tilde{\boldsymbol{y}}_j \right)_+^p \right) \right| \tag{69}$$

$$\leq \sqrt{\frac{\nu_{\max}}{mn_b}} (d+1) a_{\max} p \left\| (\boldsymbol{w}_0)_r \right\|_2^p, \tag{70}$$

$$\left\| \frac{\partial h_j (\boldsymbol{w}, \boldsymbol{v})}{\partial (\boldsymbol{v}_{\ell'})_r} \right\|_2 = 0 \text{ if } \ell' \in [L]. \tag{71}$$

**Hoeffding's inequalities for $s_{i,r}$, $(\boldsymbol{\mu}_\ell)_{r,(\alpha,\beta),i}$, and $h_{j,r}$.** For each $i \in [n_o]$ and $r \in [m]$, let

$$s_i (\boldsymbol{w}, \boldsymbol{v}) = \sqrt{\frac{1}{n_o}} \left( \sum_{\ell=0}^L \sum_{\alpha \in I_{\xi_\ell}} \sum_{\beta \in J_{\Delta\xi_{\ell+1}}} \hat{a}_{\ell,\alpha,\beta} \frac{\partial^\beta}{\partial \boldsymbol{x}^\beta} (\phi_\ell)_\alpha - f(\boldsymbol{x}_i) \right) \tag{72}$$

$$= \sum_{r=1}^m s_{i,r} (\boldsymbol{w}, \boldsymbol{v}) - \frac{1}{\sqrt{n_o}} f(\boldsymbol{x}_i), \tag{73}$$

where

$$s_{i,r} (\boldsymbol{w}, \boldsymbol{v}) = \frac{1}{\sqrt{mn_o}} \sum_{\ell=0}^L \sum_{\alpha \in I_{\xi_\ell}} \sum_{\beta \in J_{\Delta\xi_{\ell+1}}} \hat{a}_{\ell,\alpha,\beta} \frac{\partial^\beta}{\partial \boldsymbol{x}^\beta} \left( (\boldsymbol{v}_\ell)_{r,\alpha} \left( (\boldsymbol{w}_\ell)_r^\top \boldsymbol{y} \right)_+^p \right). \tag{74}$$

As $s_{i,r}$ is a $p$-th degree polynomial of $(\boldsymbol{w}_\ell)_r$, we have

$$|s_{i,r} (\boldsymbol{w}, \boldsymbol{v})| \leq \frac{a_{\max}}{\sqrt{mn_o}} \sum_{\ell=0}^L \sum_{\alpha \in I_{\xi_\ell}} \sum_{\beta \in J_{\Delta\xi_{\ell+1}}} \left| \frac{\partial^\beta}{\partial \boldsymbol{x}^\beta} \left( (\boldsymbol{v}_\ell)_{r,\alpha} \left( (\boldsymbol{w}_\ell)_r^\top \boldsymbol{y} \right)_+^p \right) \right| \tag{75}$$

$$\leq \frac{a_{\max}}{\sqrt{mn_o}} \sum_{\ell=0}^L \sum_{\alpha \in I_{\xi_\ell}} \sum_{\beta \in J_{\Delta\xi_{\ell+1}}} \left| (\boldsymbol{v}_\ell)_{r,\alpha} \right| p^{\Delta\xi_{\ell+1}} \left\| (\boldsymbol{w}_\ell)_r \right\|_2^p, \tag{76}$$

for all $i \in [n_o]$ and $r \in [m]$.

Since $\left\| (\boldsymbol{w}_\ell)_r (0) \right\|_2 < R$ and $\left| (\boldsymbol{v}_\ell)_{r,\alpha} (0) \right| \leq 1$ for all $\ell = 0, \ldots, L$, $\alpha \in I_{\xi_\ell}$, and $r \in [m]$, we have

$$|s_{i,r} (\boldsymbol{w}(0), \boldsymbol{v}(0))| \leq \sqrt{\frac{1}{mn_o}} a_{\max} \sum_{\ell=0}^L \sum_{\alpha \in I_{\xi_\ell}} \sum_{\beta \in J_{\Delta\xi_{\ell+1}}} p^{\Delta\xi_{\ell+1}} R^p \tag{77}$$

$$\leq \sqrt{\frac{\nu_{\max}}{mn_o}} a_{\max} \sum_{\ell=0}^{L} |I_{\xi_\ell}| \cdot |J_{\Delta\xi_{\ell+1}}| \cdot p^{|\xi|} R^p \tag{78}$$

$$\leq \sqrt{\frac{\nu_{\max}}{mn_o}} a_{\max} \binom{d+k}{d} \binom{d+|\xi|}{d} p^{|\xi|} R^p, \tag{79}$$

for all $i \in [n_o]$ and $r \in [m]$.

Using $\mathbb{E}\left[ (\boldsymbol{v}_\ell)_{r,\alpha} \right] = 0$, we attain $\mathbb{E}\left[ s_{i,r} \left( \boldsymbol{w}(0), \boldsymbol{v}(0) \right) \right] = 0$, and Hoeffding's inequality gives

$$\mathbb{P}\left[ \left| \sum_{r=1}^{m} s_{i,r} \left( \boldsymbol{w}(0), \boldsymbol{v}(0) \right) \right| > \frac{\varepsilon}{\sqrt{n_o}} \right] \leq 2 \exp\left( -\frac{2\left( \frac{\varepsilon}{\sqrt{n_o}} \right)^2}{m \left( 2\sqrt{\frac{\nu_{\max}}{mn_o}} a_{\max} \binom{d+k}{d}\binom{d+|\xi|}{d} p^{|\xi|} R^p \right)^2} \right) \tag{80}$$

$$\leq 2 \exp\left( -\frac{\varepsilon^2}{2\nu_{\max} a_{\max}^2 \binom{d+k}{d}^2 \binom{d+|\xi|}{d}^2 p^{2|\xi|} R^{2p}} \right), \tag{81}$$

for all $\varepsilon > 0$.

Similarly, for each $\ell \in [L]$, $\alpha \in I_{\xi_{\ell-1}}$, $\beta \in I_{\Delta\xi_\ell}$, and $i \in [n_o]$,

$$(\boldsymbol{\mu}_\ell)_{(\alpha,\beta),i} (\boldsymbol{w}, \boldsymbol{v}) = \sqrt{\frac{\nu_\ell}{n_o}} \left( \frac{\partial^\beta}{\partial \boldsymbol{x}^\beta} (\phi_{\ell-1})_\alpha (\boldsymbol{x}_i; \boldsymbol{w}_{\ell-1}, \boldsymbol{v}_{\ell-1}) - \phi_\ell (\boldsymbol{x}_i; \boldsymbol{w}_\ell, \boldsymbol{v}_\ell)_{\alpha+\beta} \right) \tag{82}$$

$$= \sum_{r=1}^{m} (\boldsymbol{\mu}_\ell)_{r,(\alpha,\beta),i} (\boldsymbol{w}, \boldsymbol{v}), \tag{83}$$

where

$$(\boldsymbol{\mu}_\ell)_{r,(\alpha,\beta),i} (\boldsymbol{w}, \boldsymbol{v}) = \sqrt{\frac{\nu_\ell}{mn_o}} \left( \frac{\partial^\beta}{\partial \boldsymbol{x}^\beta} \left( (\boldsymbol{v}_{\ell-1})_{r,\alpha} \left( (\boldsymbol{w}_{\ell-1})_r^\top \boldsymbol{y}_i \right)_+^p \right) - (\boldsymbol{v}_\ell)_{r,\alpha+\beta} \left( (\boldsymbol{w}_\ell)_r^\top \boldsymbol{y}_i \right)_+^p \right). \tag{84}$$

Then, we attain that

$$\left| (\boldsymbol{\mu}_\ell)_{r,(\alpha,\beta),i} (\boldsymbol{w}, \boldsymbol{v}) \right| \leq \sqrt{\frac{\nu_\ell}{mn_o}} \left( \left| \frac{\partial^\beta}{\partial \boldsymbol{x}^\beta} \left( (\boldsymbol{v}_{\ell-1})_{r,\alpha} \left( (\boldsymbol{w}_{\ell-1})_r^\top \boldsymbol{y}_i \right)_+^p \right) \right| + \left| (\boldsymbol{v}_\ell)_{r,\alpha+\beta} \left( (\boldsymbol{w}_\ell)_r^\top \boldsymbol{y}_i \right)_+^p \right| \right) \tag{85}$$

$$\leq \sqrt{\frac{\nu_\ell}{mn_o}} \left( p^{\Delta\xi_\ell} \left| (\boldsymbol{v}_{\ell-1})_{r,\alpha} \right| \| (\boldsymbol{w}_{\ell-1})_r \|_2^p + \left| (\boldsymbol{v}_\ell)_{r,\alpha+\beta} \right| \| (\boldsymbol{w}_\ell)_r \|_2^p \right), \tag{86}$$

and

$$\left| (\boldsymbol{\mu}_\ell)_{r,(\alpha,\beta),i} (\boldsymbol{w}(0), \boldsymbol{v}(0)) \right| \leq \sqrt{\frac{\nu_{\max}}{mn_o}} \left( p^{|\xi|} + 1 \right) R^p. \tag{87}$$

$\mathbb{E}\left[ (\boldsymbol{v}_\ell)_{r,\alpha} \right] = 0$ implies $\mathbb{E}\left[ (\boldsymbol{\mu}_\ell)_{r,(\alpha,\beta),i} \right] = 0$ and for each $i \in [n_o]$, Hoeffding's inequality with the sum over $r \in [m]$ of variables

$$X_{r,i} (\boldsymbol{w}(0), \boldsymbol{v}(0)) = \sum_{\alpha \in I_{\xi_{\ell-1}}} \sum_{\beta \in I_{\Delta\xi_\ell}} (\boldsymbol{\mu}_\ell)_{r,(\alpha,\beta),i} (\boldsymbol{w}(0), \boldsymbol{v}(0)), \tag{88}$$

with

$$|X_{r,i} (\boldsymbol{w}(0), \boldsymbol{v}(0))| \leq |I_{\xi_{\ell-1}}| \cdot |I_{\Delta\xi_\ell}| \sqrt{\frac{\nu_{\max}}{mn_o}} \left( p^{|\xi|} + 1 \right) R^p \tag{89}$$

$$< \sqrt{\frac{\nu_{\max}}{mn_o}} \binom{d+k}{d} \binom{d+|\xi|}{d} \left( p^{|\xi|} + 1 \right) R^p \tag{90}$$

gives

$$\mathbb{P}\left[\left|\sum_{r=1}^{m}(\boldsymbol{\mu}_\ell)_{r,(\alpha,\beta),i}\left(\boldsymbol{w}\left(0\right),\boldsymbol{v}\left(0\right)\right)\right|>\frac{\varepsilon}{\sqrt{n_o}}\right] \tag{91}$$

$$\leq 2\exp\left(-\frac{2\left(\frac{\varepsilon}{\sqrt{n_o}}\right)^2}{m\left(\sqrt{\frac{\nu_{\max}}{mn_o}}\binom{d+k}{d}\binom{d+|\xi|}{d}\left(p^{|\xi|}+1\right)R^p\right)^2}\right) \tag{92}$$

$$\leq 2\exp\left(-\frac{\varepsilon^2}{\nu_{\max}\binom{d+k}{d}^2\binom{d+|\xi|}{d}^2\left(p^{|\xi|}+1\right)^2R^{2p}}\right). \tag{93}$$

For each $j\in[n_b]$, let us denote

$$h_j\left(\boldsymbol{w},\boldsymbol{v}\right)=\sqrt{\frac{\nu}{n_b}}\sum_{\alpha\in J_1}\tilde{a}_\alpha\phi_0\left(\tilde{\boldsymbol{x}}_j;\boldsymbol{w}_0,\boldsymbol{v}_0\right)-\sqrt{\frac{\nu}{n_b}}g\left(\tilde{\boldsymbol{x}}_j\right) \tag{94}$$

$$=\sum_{r=1}^{m}h_{j,r}\left(\left(\boldsymbol{w}_0\right)_r,\left(\boldsymbol{v}_0\right)_r\right)-\sqrt{\frac{\nu}{n_b}}g\left(\tilde{\boldsymbol{x}}_j\right), \tag{95}$$

where

$$h_{j,r}\left(\boldsymbol{w},\boldsymbol{v}\right)=\sqrt{\frac{\nu}{mn_b}}\sum_{\alpha\in J_1}\tilde{a}_\alpha\frac{\partial^\alpha}{\partial\boldsymbol{x}^\alpha}\left((\boldsymbol{v}_0)_{r,\boldsymbol{0}}\cdot\left((\boldsymbol{w}_0)_r^\top\tilde{\boldsymbol{y}}_j\right)_+^p\right). \tag{96}$$

It is clear that $h_{jr}$ is expressed as a $p$-th order polynomial of $(\boldsymbol{w}_0)_r$ and we can deduce that

$$\left|h_{j,r}\left(\boldsymbol{w}\left(0\right),\boldsymbol{v}\left(0\right)\right)\right|\leq\sqrt{\frac{\nu_{\max}}{mn_b}}a_{\max}\left(d+1\right)pR^p, \tag{97}$$

for all $j\in[n_b]$ and $r\in[m]$.

Similar to the case of $s_{i,r}$ and $(\boldsymbol{\mu}_\ell)_{r,(\alpha,\beta),i}$, Hoeffding's inequality gives

$$\mathbb{P}\left[\left|\sum_{r=1}^{m}h_{j,r}\left(\boldsymbol{w}\left(0\right),\boldsymbol{v}\left(0\right)\right)\right|>\frac{\varepsilon}{\sqrt{n_b}}\right]\leq 2\exp\left(-\frac{\varepsilon^2}{2\nu_{\max}a_{\max}^2\left(d+1\right)^2p^2R^{2p}}\right). \tag{98}$$

**Components of $\boldsymbol{Q}_{\ell_1,\ell_2}$**   Each component of $\boldsymbol{Q}_{\ell_1,\ell_2}$ is polynomial of order $p$ as follows:

$$\left(\boldsymbol{Q}_{0,0}\right)_{i_1,i_2} \tag{99}$$

$$=\sum_{\ell=0}^{L}\sum_{r=1}^{m}\sum_{\alpha\in I_{\xi_\ell}}\left(\frac{\partial}{\partial\left(\boldsymbol{v}_\ell\right)_{r,\alpha}}\boldsymbol{s}_{i_1}\right)\left(\frac{\partial}{\partial\left(\boldsymbol{v}_\ell\right)_{r,\alpha}}\boldsymbol{s}_{i_2}\right) \tag{100}$$

$$=\frac{1}{mn_o}\sum_{\ell=0}^{L}\sum_{r=1}^{m}\sum_{\alpha\in I_{\xi_\ell}}\left(\sum_{\beta\in J_{\Delta\xi_{\ell+1}}}\hat{a}_{\ell,\alpha,\beta}\frac{\partial^\beta}{\partial\boldsymbol{x}^\beta}\left((\boldsymbol{w}_\ell)_r^\top\boldsymbol{y}_{i_1}\right)_+^p\right)\left(\sum_{\beta\in J_{\Delta\xi_{\ell+1}}}\hat{a}_{\ell,\alpha,\beta}\frac{\partial^\beta}{\partial\boldsymbol{x}^\beta}\left((\boldsymbol{w}_\ell)_r^\top\boldsymbol{y}_{i_2}\right)_+^p\right). \tag{101}$$

$$\left(\boldsymbol{Q}_{0,\ell}\right)_{i_1,((\alpha,\beta),i_2)} \tag{102}$$

$$=\sum_{\ell'=0}^{L}\sum_{r=1}^{m}\sum_{\alpha'\in I_{\xi_{\ell'}}}\left(\frac{\partial}{\partial\left(\boldsymbol{v}_{\ell'}\right)_{r,\alpha'}}\boldsymbol{s}_{i_1}\right)\left(\frac{\partial}{\partial\left(\boldsymbol{v}_{\ell'}\right)_{r,\alpha'}}(\boldsymbol{\mu}_\ell)_{(\alpha,\beta),i_2}\right) \tag{103}$$

$$=\sum_{r=1}^{m}\sum_{\alpha'\in I_{\xi_{\ell-1}}}\left(\frac{\partial}{\partial\left(\boldsymbol{v}_{\ell-1}\right)_{r,\alpha'}}\boldsymbol{s}_{i_1}\right)\left(\frac{\partial}{\partial\left(\boldsymbol{v}_{\ell-1}\right)_{r,\alpha'}}(\boldsymbol{\mu}_\ell)_{(\alpha,\beta),i_2}\right) \tag{104}$$

$$+ \sum_{r=1}^{m} \sum_{\alpha' \in I_{\xi_\ell}} \left( \frac{\partial}{\partial (\boldsymbol{v}_\ell)_{r,\alpha'}} s_{i_1} \right) \left( \frac{\partial}{\partial (\boldsymbol{v}_\ell)_{r,\alpha'}} (\boldsymbol{\mu}_\ell)_{(\alpha,\beta),i_2} \right) \tag{105}$$

$$= \sum_{r=1}^{m} \sum_{\alpha' \in I_{\xi_{\ell-1}}} \left( \frac{\partial}{\partial (\boldsymbol{v}_{\ell-1})_{r,\alpha'}} s_{i_1} \right) \left( \frac{\partial}{\partial (\boldsymbol{v}_{\ell-1})_{r,\alpha'}} (\boldsymbol{\mu}_\ell)_{(\alpha,\beta),i_2} \right) \tag{106}$$

$$+ \sum_{r=1}^{m} \left( \frac{\partial}{\partial (\boldsymbol{v}_\ell)_{r,\alpha+\beta}} s_{i_1} \right) \left( \frac{\partial}{\partial (\boldsymbol{v}_\ell)_{r,\alpha+\beta}} (\boldsymbol{\mu}_\ell)_{(\alpha,\beta),i_2} \right) \tag{107}$$

$$= \frac{\sqrt{\nu_\ell}}{mn_o} \sum_{r=1}^{m} \left( \left( \sum_{\beta' \in J_{\Delta\xi_\ell}} \hat{a}_{\ell-1,\alpha,\beta'} \frac{\partial^{\beta'}}{\partial \boldsymbol{x}^{\beta'}} \left( (\boldsymbol{w}_{\ell-1})_r^\top \boldsymbol{y}_{i_1} \right)_+^p \right) \left( \frac{\partial^\beta}{\partial \boldsymbol{x}^\beta} \left( (\boldsymbol{w}_{\ell-1})_r^\top \boldsymbol{y}_{i_2} \right)_+^p \right) \tag{108}$$

$$- \left( \sum_{\beta' \in J_{\Delta\xi_{\ell+1}}} \hat{a}_{\ell,\alpha,\beta'} \frac{\partial^{\beta'}}{\partial \boldsymbol{x}^{\beta'}} \left( (\boldsymbol{w}_\ell)_r^\top \boldsymbol{y}_{i_1} \right)_+^p \right) \left( (\boldsymbol{w}_\ell)_r^\top \boldsymbol{y}_{i_2} \right)_+^p \right). \tag{109}$$

$$(\boldsymbol{Q}_{0,L+1})_{i,j} = \sum_{\ell=0}^{L} \sum_{r=1}^{m} \sum_{\alpha \in I_{\xi_\ell}} \left( \frac{\partial}{\partial (\boldsymbol{v}_\ell)_{r,\alpha}} s_i \right) \left( \frac{\partial}{\partial (\boldsymbol{v}_\ell)_{r,\alpha}} h_j \right) \tag{110}$$

$$= \sum_{r=1}^{m} \left( \frac{\partial}{\partial (\boldsymbol{v}_0)_{r,1}} s_i \right) \left( \frac{\partial}{\partial (\boldsymbol{v}_0)_{r,1}} h_j \right) \tag{111}$$

$$= \frac{1}{m} \sqrt{\frac{\nu}{n_o n_b}} \sum_{r=1}^{m} \left( \sum_{\beta \in J_{\Delta\xi_1}} \hat{a}_{0,1,\beta} \frac{\partial^\beta}{\partial \boldsymbol{x}^\beta} \left( (\boldsymbol{w}_0)_r^\top \boldsymbol{y}_i \right)_+^p \right) \left( \sum_{\alpha \in J_1} \tilde{a}_\alpha \frac{\partial^\alpha}{\partial \boldsymbol{x}^\alpha} \left( (\boldsymbol{w}_0)_r^\top \tilde{\boldsymbol{y}}_j \right)_+^p \right). \tag{112}$$

$$(\boldsymbol{Q}_{\ell,\ell})_{((\alpha_1,\beta_1),i_1),((\alpha_2,\beta_2),i_2)} \tag{113}$$

$$= \sum_{\ell'=0}^{L} \sum_{r=1}^{m} \sum_{\alpha \in I_{\xi_{\ell'}}} \left( \frac{\partial}{\partial (\boldsymbol{v}_{\ell'})_{r,\alpha}} (\boldsymbol{\mu}_\ell)_{(\alpha_1,\beta_1),i_1} \right) \left( \frac{\partial}{\partial (\boldsymbol{v}_{\ell'})_{r,\alpha}} (\boldsymbol{\mu}_\ell)_{(\alpha_2,\beta_2),i_2} \right) \tag{114}$$

$$= \sum_{r=1}^{m} \sum_{\alpha \in I_{\xi_{\ell-1}}} \left( \frac{\partial}{\partial (\boldsymbol{v}_{\ell-1})_{r,\alpha}} (\boldsymbol{\mu}_\ell)_{(\alpha_1,\beta_1),i_1} \right) \left( \frac{\partial}{\partial (\boldsymbol{v}_{\ell-1})_{r,\alpha}} (\boldsymbol{\mu}_\ell)_{(\alpha_2,\beta_2),i_2} \right) \tag{115}$$

$$+ \sum_{r=1}^{m} \sum_{\alpha \in I_{\xi_\ell}} \left( \frac{\partial}{\partial (\boldsymbol{v}_\ell)_{r,\alpha}} (\boldsymbol{\mu}_\ell)_{(\alpha_1,\beta_1),i_1} \right) \left( \frac{\partial}{\partial (\boldsymbol{v}_\ell)_{r,\alpha}} (\boldsymbol{\mu}_\ell)_{(\alpha_2,\beta_2),i_2} \right) \tag{116}$$

$$= \sum_{r=1}^{m} \left( \sqrt{\frac{\nu_\ell}{mn_o}} \frac{\partial^{\beta_1}}{\partial \boldsymbol{x}^{\beta_1}} \left( (\boldsymbol{w}_{\ell-1})_r^\top \boldsymbol{y}_{i_1} \right)_+^p \right) \left( \sqrt{\frac{\nu_\ell}{mn_o}} \frac{\partial^{\beta_2}}{\partial \boldsymbol{x}^{\beta_2}} \left( (\boldsymbol{w}_{\ell-1})_r^\top \boldsymbol{y}_{i_2} \right)_+^p \right) \mathbf{1}_{\{\alpha_1=\alpha_2\}} \tag{117}$$

$$+ \sum_{r=1}^{m} \left( -\sqrt{\frac{\nu_\ell}{mn_o}} \left( (\boldsymbol{w}_\ell)_r^\top \boldsymbol{y}_{i_1} \right)_+^p \right) \left( -\sqrt{\frac{\nu_\ell}{mn_o}} \left( (\boldsymbol{w}_\ell)_r^\top \boldsymbol{y}_{i_2} \right)_+^p \right) \mathbf{1}_{\{\alpha_1+\beta_1=\alpha_2+\beta_2\}} \tag{118}$$

$$= \frac{\nu_\ell}{mn_o} \sum_{r=1}^{m} \left( \left( \frac{\partial^{\beta_1}}{\partial \boldsymbol{x}^{\beta_1}} \left( (\boldsymbol{w}_{\ell-1})_r^\top \boldsymbol{y}_{i_1} \right)_+^p \right) \left( \frac{\partial^{\beta_2}}{\partial \boldsymbol{x}^{\beta_2}} \left( (\boldsymbol{w}_{\ell-1})_r^\top \boldsymbol{y}_{i_2} \right)_+^p \right) \mathbf{1}_{\{\alpha_1=\alpha_2\}} \tag{119}$$

$$+ \left( (\boldsymbol{w}_\ell)_r^\top \boldsymbol{y}_{i_1} \right)_+^p \left( (\boldsymbol{w}_\ell)_r^\top \boldsymbol{y}_{i_2} \right)_+^p \mathbf{1}_{\{\alpha_1+\beta_1=\alpha_2+\beta_2\}} \right). \tag{120}$$

$$(\boldsymbol{Q}_{\ell,\ell+1})_{((\alpha_1,\beta_1),i_1),((\alpha_2,\beta_2),i_2)} \tag{121}$$

$$= \sum_{\ell'=0}^{L} \sum_{r=1}^{m} \sum_{\alpha \in I_{\xi_{\ell'}}} \left( \frac{\partial}{\partial (\boldsymbol{v}_{\ell'})_{r,\alpha}} (\boldsymbol{\mu}_{\ell})_{(\alpha_1,\beta_1),i_1} \right) \left( \frac{\partial}{\partial (\boldsymbol{v}_{\ell'})_{r,\alpha}} (\boldsymbol{\mu}_{\ell+1})_{(\alpha_2,\beta_2),i_2} \right) \tag{122}$$

$$= \sum_{r=1}^{m} \sum_{\alpha \in I_{\xi_{\ell}}} \left( \frac{\partial}{\partial (\boldsymbol{v}_{\ell})_{r,\alpha}} (\boldsymbol{\mu}_{\ell})_{(\alpha_1,\beta_1),i_1} \right) \left( \frac{\partial}{\partial (\boldsymbol{v}_{\ell})_{r,\alpha}} (\boldsymbol{\mu}_{\ell+1})_{(\alpha_2,\beta_2),i_2} \right) \tag{123}$$

$$= \sum_{r=1}^{m} \sum_{\alpha \in I_{\xi_{\ell}}} \left( -\sqrt{\frac{\nu_{\ell}}{mn_o}} \left( (\boldsymbol{w}_{\ell})_r^\top \boldsymbol{y}_{i_1} \right)_+^p \mathbf{1}_{\{\alpha=\alpha_1+\beta_1\}} \right) \left( \sqrt{\frac{\nu_{\ell+1}}{mn_o}} \frac{\partial^{\beta_2}}{\partial \boldsymbol{x}^{\beta_2}} \left( (\boldsymbol{w}_{\ell})_r^\top \boldsymbol{y}_{i_2} \right)_+^p \right) \mathbf{1}_{\{\alpha=\alpha_2\}} \tag{124}$$

$$= \frac{\sqrt{\nu_{\ell}\nu_{\ell+1}}}{mn_o} \sum_{r=1}^{m} \left( -\left( (\boldsymbol{w}_{\ell})_r^\top \boldsymbol{y}_{i_1} \right)_+^p \right) \left( \frac{\partial^{\beta_2}}{\partial \boldsymbol{x}^{\beta_2}} \left( (\boldsymbol{w}_{\ell})_r^\top \boldsymbol{y}_{i_2} \right)_+^p \right) \mathbf{1}_{\{\alpha_2=\alpha_1+\beta_1\}}. \tag{125}$$

$$\left( \boldsymbol{Q}_{\ell,L+1} \right)_{((\alpha,\beta),i),j} \tag{126}$$

$$= \sum_{\ell'=0}^{L} \sum_{r=1}^{m} \sum_{\alpha' \in I_{\xi_{\ell'}}} \left( \frac{\partial}{\partial (\boldsymbol{v}_{\ell'})_{r,\alpha'}} (\boldsymbol{\mu}_{\ell})_{(\alpha,\beta),i} \right) \left( \frac{\partial}{\partial (\boldsymbol{v}_{\ell'})_{r,\alpha'}} h_j \right) \tag{127}$$

$$= \sum_{r=1}^{m} \left( \frac{\partial}{\partial (\boldsymbol{v}_0)_{r,1}} (\boldsymbol{\mu}_{\ell})_{(\alpha,\beta),i} \right) \left( \frac{\partial}{\partial (\boldsymbol{v}_0)_{r,1}} h_j \right) \tag{128}$$

$$= \sum_{r=1}^{m} \left( \sqrt{\frac{\nu_1}{mn_o}} \frac{\partial^{\beta}}{\partial \boldsymbol{x}^{\beta}} \left( (\boldsymbol{w}_0)_r^\top \boldsymbol{y}_i \right)_+^p \right) \left( \sqrt{\frac{\nu}{mn_b}} \sum_{\alpha \in J_1} \tilde{a}_{\alpha} \left( (\boldsymbol{w}_0)_r^\top \tilde{\boldsymbol{y}}_j \right)_+^p \right) \mathbf{1}_{\{\ell=1\}} \tag{129}$$

$$= \frac{1}{m} \sqrt{\frac{\nu_1 \nu}{n_o n_b}} \sum_{r=1}^{m} \left( \frac{\partial^{\beta}}{\partial \boldsymbol{x}^{\beta}} \left( (\boldsymbol{w}_0)_r^\top \boldsymbol{y}_i \right)_+^p \right) \left( \sum_{\alpha \in J_1} \tilde{a}_{\alpha} \frac{\partial^{\alpha}}{\partial \boldsymbol{x}^{\alpha}} \left( (\boldsymbol{w}_0)_r^\top \tilde{\boldsymbol{y}}_j \right)_+^p \right) \mathbf{1}_{\{\ell=1\}}. \tag{130}$$

$$\left( \boldsymbol{Q}_{L+1,L+1} \right)_{j_1,j_2} \tag{131}$$

$$= \sum_{\ell=0}^{L} \sum_{r=1}^{m} \sum_{\alpha \in I_{\xi_{\ell}}} \left( \frac{\partial}{\partial (\boldsymbol{v}_{\ell})_{r,\alpha}} h_{j_1} \right) \left( \frac{\partial}{\partial (\boldsymbol{v}_{\ell})_{r,\alpha}} h_{j_2} \right) \tag{132}$$

$$= \sum_{r=1}^{m} \left( \frac{\partial}{\partial (\boldsymbol{v}_0)_{r,1}} h_{j_1} \right) \left( \frac{\partial}{\partial (\boldsymbol{v}_0)_{r,1}} h_{j_2} \right) \tag{133}$$

$$= \sum_{r=1}^{m} \left( \sqrt{\frac{\nu}{mn_b}} \sum_{\alpha \in J_1} \tilde{a}_{\alpha} \frac{\partial^{\alpha}}{\partial \boldsymbol{x}^{\alpha}} \left( (\boldsymbol{w}_0)_r^\top \tilde{\boldsymbol{y}}_{j_1} \right)_+^p \right) \left( \sqrt{\frac{\nu}{mn_b}} \sum_{\alpha \in J_1} \tilde{a}_{\alpha} \frac{\partial^{\alpha}}{\partial \boldsymbol{x}^{\alpha}} \left( (\boldsymbol{w}_0)_r^\top \tilde{\boldsymbol{y}}_{j_2} \right)_+^p \right) \tag{134}$$

$$= \frac{\nu}{mn_b} \sum_{r=1}^{m} \left( \sum_{\alpha \in J_1} \tilde{a}_{\alpha} \frac{\partial^{\alpha}}{\partial \boldsymbol{x}^{\alpha}} \left( (\boldsymbol{w}_0)_r^\top \tilde{\boldsymbol{y}}_{j_1} \right)_+^p \right) \left( \sum_{\alpha \in J_1} \tilde{a}_{\alpha} \frac{\partial^{\alpha}}{\partial \boldsymbol{x}^{\alpha}} \left( (\boldsymbol{w}_0)_r^\top \tilde{\boldsymbol{y}}_{j_2} \right)_+^p \right). \tag{135}$$

**The number of possible nonzero elements in $Q_{\ell_1,\ell_2}$ and $\widehat{G}_{\boldsymbol{v}}$.** There are possibly $n_o^2$, $\left| I_{\xi_{\ell-1}} \right| \left| I_{\Delta\xi_{\ell}} \right| n_o^2$, and $n_o n_b$ nonzero elements in $\boldsymbol{Q}_{0,0}$, $\boldsymbol{Q}_{0,\ell}$, and $\boldsymbol{Q}_{0,L+1}$, respectively. For each $(\alpha_1,\beta_1) \in I_{\xi_{\ell-1}} \times I_{\Delta\xi_{\ell}}$, there are at most $|I_{\Delta\xi_{\ell}}|$ pairs of $(\alpha_2,\beta_2) \in I_{\xi_{\ell-1}} \times I_{\Delta\xi_{\ell}}$ such that $\alpha_1 + \beta_1 = \alpha_2 + \beta_2$ and at most $|I_{\Delta\xi_{\ell}}|$ pairs of $(\alpha_2,\beta_2) \in I_{\xi_{\ell-1}} \times I_{\Delta\xi_{\ell}}$ such that $\alpha_1 = \alpha_2$. Thus, there are at most $2|I_{\Delta\xi_{\ell}}|$ pairs of $(\alpha_2,\beta_2) \in I_{\xi_{\ell-1}} \times I_{\Delta\xi_{\ell}}$ with $\alpha_1 + \beta_1 = \alpha_2 + \beta_2$ or $\alpha_1 = \alpha_2$, and $\boldsymbol{Q}_{\ell,\ell}$ has possibly $\left( \left| I_{\xi_{\ell-1}} \right| |I_{\Delta\xi_{\ell}}| n_o \right) \left( 2|I_{\Delta\xi_{\ell}}| n_o \right) = 2 \left| I_{\xi_{\ell-1}} \right| \cdot |I_{\Delta\xi_{\ell}}|^2 n_o^2$ nonzero elements. Similarly, each $(\alpha_1,\beta_1) \in I_{\xi_{\ell-1}} \times I_{\Delta\xi_{\ell}}$ uniquely determines $\alpha_2 = \alpha_1 + \beta_1 \in I_{\xi_{\ell}}$, there are $|I_{\Delta\xi_{\ell+1}}|$ pairs of $(\alpha_2,\beta_2) \in I_{\xi_{\ell}} \times I_{\Delta\xi_{\ell+1}}$ with $\alpha_2 = \alpha_1 + \beta_1$. Hence, $\boldsymbol{Q}_{\ell,\ell+1}$ has $\left| I_{\xi_{\ell-1}} \right| \cdot |I_{\Delta\xi_{\ell}}| \cdot |I_{\Delta\xi_{\ell+1}}| n_o^2$ nonzero elements. $\boldsymbol{Q}_{1,L+1}$ has $|I_{\Delta\xi_1}| n_o n_b$ nonzero elements. The number of nonzero elements in $\boldsymbol{Q}_{L+1,L+1}$ is $n_b^2$. After all, the number of nonzero elements in $\widehat{G}_{\boldsymbol{v}}$ is

| Matrix | The number of nonzero elements |
|---|---|
| $\boldsymbol{Q}_{0,0}$ | $n_o^2$ |
| $\boldsymbol{Q}_{0,\ell}$ | $\left|I_{\xi_{\ell-1}}\right|\left|I_{\Delta\xi_\ell}\right|n_o^2$ |
| $\boldsymbol{Q}_{0,L+1}$ | $n_o n_b$ |
| $\boldsymbol{Q}_{\ell,\ell}$ | $2\left|I_{\xi_{\ell-1}}\right|\cdot\left|I_{\Delta\xi_\ell}\right|^2 n_o^2$ |
| $\boldsymbol{Q}_{\ell,\ell+1}$ | $\left|I_{\xi_{\ell-1}}\right|\cdot\left|I_{\Delta\xi_\ell}\right|\cdot\left|I_{\Delta\xi_{\ell+1}}\right|n_o^2$ |
| $\boldsymbol{Q}_{1,L+1}$ | $\left|I_{\Delta\xi_1}\right|n_o n_b$ |
| $\boldsymbol{Q}_{L+1,L+1}$ | $n_b^2$ |

Table 1: The number of nonzero elements in blocks in $\boldsymbol{G}$.

at most $N_1 = 7\binom{d+k}{d}\binom{d+|\xi|}{d}^2 n_o^2 + 4\binom{d+|\xi|}{d}n_o n_b + n_b^2$, because

$$\left(\text{the number of nonzero elements of } \widehat{\boldsymbol{G}}_{\boldsymbol{v}}\right) \tag{136}$$

$$= \sum_{\ell=0}^{L+1} \left(\text{the number of nonzero elements in } \boldsymbol{Q}_{\ell,\ell}\right) \tag{137}$$

$$+ 2\sum_{\ell=0}^{L}\sum_{\ell'=\ell+1}^{L+1} \left(\text{the number of nonzero elements in } \boldsymbol{Q}_{\ell,\ell'}\right) \tag{138}$$

$$\leq n_o^2 + \sum_{\ell=1}^{L}\left(2\left|I_{\xi_{\ell-1}}\right|\left|I_{\Delta\xi_\ell}\right|^2 n_o^2\right) + n_b^2 \tag{139}$$

$$+ 2\left\{\sum_{\ell=1}^{L}\left(\left|I_{\xi_{\ell-1}}\right|\left|I_{\Delta\xi_\ell}\right|n_o^2\right) + n_o n_b + \sum_{\ell=1}^{L}\left(\left|I_{\xi_{\ell-1}}\right|\left|I_{\Delta\xi_\ell}\right|\left|I_{\Delta\xi_{\ell+1}}\right|n_o^2\right) + \left|I_{\Delta\xi_1}\right|n_o n_b\right\} \tag{140}$$

$$\leq n_o^2 + 2\binom{d+|\xi|}{d}^2\sum_{\ell=1}^{L}\left|I_{\xi_{\ell-1}}\right|n_o^2 + n_b^2 \tag{141}$$

$$+ 2\left\{\binom{d+|\xi|}{d}\sum_{\ell=1}^{L}\left|I_{\xi_{\ell-1}}\right|n_o^2 + n_o n_b + \binom{d+|\xi|}{d}^2\sum_{\ell=1}^{L}\left|I_{\xi_{\ell-1}}\right|n_o^2 + \binom{d+|\xi|}{d}n_o n_b\right\} \tag{142}$$

$$\leq \left(1 + 2\binom{d+|\xi|}{d}\binom{d+k}{d} + 4\binom{d+|\xi|}{d}^2\binom{d+k}{d}\right)n_o^2 + 2\left(1 + \binom{d+|\xi|}{d}\right)n_o n_b + n_b^2 \tag{143}$$

$$\leq 7\binom{d+k}{d}\binom{d+|\xi|}{d}^2 n_o^2 + 4\binom{d+|\xi|}{d}n_o n_b + n_b^2 \tag{144}$$

$$= N_1. \tag{145}$$

**Norms of partial derivative of $\boldsymbol{Q}_{\ell_1,\ell_2}$.** For each $i_1, i_2 \in [n_o]$, $r \in [m]$, and $\ell \in \{0\}\cup[L]$, we have

$$\left(\boldsymbol{Q}_{0,0}\right)_{i_1,i_2} \tag{146}$$

$$= \frac{1}{mn_o}\sum_{\ell=0}^{L}\sum_{r=1}^{m}\sum_{\alpha\in I_{\xi_\ell}}\left(\sum_{\beta\in J_{\Delta\xi_{\ell+1}}}\hat{a}_{\ell,\alpha,\beta}\frac{\partial^\beta}{\partial\boldsymbol{x}^\beta}\left(\left(\boldsymbol{w}_\ell\right)_r^\top\boldsymbol{y}_{i_1}\right)_+^p\right)\left(\sum_{\beta\in J_{\Delta\xi_{\ell+1}}}\hat{a}_{\ell,\alpha,\beta}\frac{\partial^\beta}{\partial\boldsymbol{x}^\beta}\left(\left(\boldsymbol{w}_\ell\right)_r^\top\boldsymbol{y}_{i_2}\right)_+^p\right), \tag{147}$$

$$\left\|\frac{\partial}{\partial\left(\boldsymbol{w}_\ell\right)_r}\left(\boldsymbol{Q}_{0,0}\right)_{i_1,i_2}\right\|_2 \tag{148}$$

$$= \left\| \frac{1}{mn_o} \frac{\partial}{\partial (\boldsymbol{w}_\ell)_r} \left\{ \sum_{\alpha \in I_{\xi_\ell}} \left( \sum_{\beta \in J_{\Delta \xi_{\ell+1}}} \hat{a}_{\ell,\alpha,\beta} \frac{\partial^\beta}{\partial \boldsymbol{x}^\beta} \left( (\boldsymbol{w}_\ell)_r^\top \boldsymbol{y}_{i_1} \right)_+^p \right) \left( \sum_{\beta \in J_{\Delta \xi_{\ell+1}}} \hat{a}_{\ell,\alpha,\beta} \frac{\partial^\beta}{\partial \boldsymbol{x}^\beta} \left( (\boldsymbol{w}_\ell)_r^\top \boldsymbol{y}_{i_2} \right)_+^p \right) \right\} \right\|_2 \tag{149}$$

$$\leq \frac{1}{mn_o} |I_{\xi_\ell}| \, a_{\max}^2 \left\| \frac{\partial}{\partial (\boldsymbol{w}_\ell)_r} \left\{ \left( \sum_{\beta \in J_{\Delta \xi_{\ell+1}}} \frac{\partial^\beta}{\partial \boldsymbol{x}^\beta} \left( (\boldsymbol{w}_\ell)_r^\top \boldsymbol{y}_{i_1} \right)_+^p \right) \left( \sum_{\beta \in J_{\Delta \xi_{\ell+1}}} \frac{\partial^\beta}{\partial \boldsymbol{x}^\beta} \left( (\boldsymbol{w}_\ell)_r^\top \boldsymbol{y}_{i_2} \right)_+^p \right) \right\} \right\|_2 \tag{150}$$

$$\leq \frac{1}{mn_o} a_{\max}^2 |I_{\xi_\ell}| \left| J_{\Delta \xi_{\ell+1}} \right|^2 p^{2|\xi|+1} \| (\boldsymbol{w}_\ell)_r \|_2^{2p-1}, \tag{151}$$

$$\frac{\partial}{\partial (\boldsymbol{v}_\ell)_r} \left( \boldsymbol{Q}_{0,0} \right)_{i_1,i_2} = \boldsymbol{0}. \tag{152}$$

For each $\ell \in [L]$, $\alpha \in I_{\xi_{\ell-1}}$, $\beta \in I_{\Delta \xi_\ell}$, and $i_1, i_2 \in [n_o]$,

$$\left( \boldsymbol{Q}_{0,\ell} \right)_{i_1,((\alpha,\beta),i_2)} \tag{153}$$

$$= \frac{\sqrt{\nu_\ell}}{mn_o} \sum_{r=1}^m \left( \left( \sum_{\beta' \in J_{\Delta \xi_\ell}} \hat{a}_{\ell-1,\alpha,\beta'} \frac{\partial^{\beta'}}{\partial \boldsymbol{x}^{\beta'}} \left( (\boldsymbol{w}_{\ell-1})_r^\top \boldsymbol{y}_{i_1} \right)_+^p \right) \left( \frac{\partial^\beta}{\partial \boldsymbol{x}^\beta} \left( (\boldsymbol{w}_{\ell-1})_r^\top \boldsymbol{y}_{i_2} \right)_+^p \right) \right. \tag{154}$$

$$\left. - \left( \sum_{\beta' \in J_{\Delta \xi_{\ell+1}}} \hat{a}_{\ell,\alpha,\beta'} \frac{\partial^{\beta'}}{\partial \boldsymbol{x}^{\beta'}} \left( (\boldsymbol{w}_\ell)_r^\top \boldsymbol{y}_{i_1} \right)_+^p \right) \left( (\boldsymbol{w}_\ell)_r^\top \boldsymbol{y}_{i_2} \right)_+^p \right), \tag{155}$$

$$\left\| \frac{\partial}{\partial (\boldsymbol{w}_{\ell-1})_r} \left( \boldsymbol{Q}_{0,\ell} \right)_{i_1,((\alpha,\beta),i_2)} \right\|_2 \tag{156}$$

$$= \left\| \frac{\sqrt{\nu_\ell}}{mn_o} \frac{\partial}{\partial (\boldsymbol{w}_{\ell-1})_r} \left\{ \left( \sum_{\beta' \in J_{\Delta \xi_\ell}} \hat{a}_{\ell-1,\alpha,\beta'} \frac{\partial^{\beta'}}{\partial \boldsymbol{x}^{\beta'}} \left( (\boldsymbol{w}_{\ell-1})_r^\top \boldsymbol{y}_{i_1} \right)_+^p \right) \left( \frac{\partial^\beta}{\partial \boldsymbol{x}^\beta} \left( (\boldsymbol{w}_{\ell-1})_r^\top \boldsymbol{y}_{i_2} \right)_+^p \right) \right\} \right\|_2 \tag{157}$$

$$\leq \frac{\sqrt{\nu_\ell}}{mn_o} a_{\max} |J_{\Delta \xi_\ell}| \, p^{2\Delta \xi_\ell +1} \| (\boldsymbol{w}_{\ell-1})_r \|_2^{2p-1} \tag{158}$$

$$\leq \frac{\sqrt{\nu_{\max}}}{mn_o} a_{\max} \binom{d+|\xi|}{d} p^{2|\xi|+1} \| (\boldsymbol{w}_{\ell-1})_r \|_2^{2p-1}, \tag{159}$$

and

$$\left\| \frac{\partial}{\partial (\boldsymbol{w}_\ell)_r} \left( \boldsymbol{Q}_{0,\ell} \right)_{i_1,((\alpha,\beta),i_2)} \right\|_2 \tag{160}$$

$$= \left\| -\frac{\sqrt{\nu_\ell}}{mn_o} \frac{\partial}{\partial (\boldsymbol{w}_\ell)_r} \left( \left( \sum_{\beta' \in J_{\Delta \xi_{\ell+1}}} \hat{a}_{\ell,\alpha,\beta'} \frac{\partial^{\beta'}}{\partial \boldsymbol{x}^{\beta'}} \left( (\boldsymbol{w}_\ell)_r^\top \boldsymbol{y}_{i_1} \right)_+^p \right) \left( (\boldsymbol{w}_\ell)_r^\top \boldsymbol{y}_{i_2} \right)_+^p \right) \right\|_2 \tag{161}$$

$$\leq \frac{\sqrt{\nu_\ell}}{mn_o} a_{\max} \left| J_{\Delta \xi_{\ell+1}} \right| p^{\Delta \xi_{\ell+1}+1} \| (\boldsymbol{w}_\ell)_r \|_2^{2p-1} \tag{162}$$

$$\leq \frac{\sqrt{\nu_{\max}}}{mn_o} a_{\max} \binom{d+|\xi|}{d} p^{2|\xi|+1} \| (\boldsymbol{w}_\ell)_r \|_2^{2p-1}, \tag{163}$$

$$\frac{\partial}{\partial (\boldsymbol{w}_{\ell'})_r} \left( \boldsymbol{Q}_{0,\ell} \right)_{i_1,((\alpha,\beta),i_2)} = \boldsymbol{0}, \text{ if } \ell' \neq \ell-1, \ell, \tag{164}$$

$$\frac{\partial}{\partial (\boldsymbol{v}_{\ell'})_r} \left( \boldsymbol{Q}_{0,\ell} \right)_{i_1,((\alpha,\beta),i_2)} = \boldsymbol{0}, \, \forall \ell' \in [L]. \tag{165}$$

For each $i \in [n_o]$ and $j \in [n_b]$,

$$\left( \boldsymbol{Q}_{0,L+1} \right)_{i,j} \tag{166}$$

$$= \frac{1}{m} \sqrt{\frac{\nu}{n_o n_b}} \sum_{r=1}^{m} \left( \sum_{\beta \in J_{\Delta \xi_1}} \hat{a}_{0,1,\beta} \frac{\partial^\beta}{\partial \boldsymbol{x}^\beta} \left( (\boldsymbol{w}_0)_r^\top \boldsymbol{y}_i \right)_+^p \right) \left( \sum_{\alpha \in J_1} \tilde{a}_\alpha \frac{\partial^\alpha}{\partial \boldsymbol{x}^\alpha} \left( (\boldsymbol{w}_0)_r^\top \tilde{\boldsymbol{y}}_j \right)_+^p \right), \tag{167}$$

$$\left\| \frac{\partial}{\partial (\boldsymbol{w}_0)_r} \left( \boldsymbol{Q}_{0,L+1} \right)_{i,j} \right\|_2 \tag{168}$$

$$= \left\| \frac{1}{m} \sqrt{\frac{\nu}{n_o n_b}} \frac{\partial}{\partial (\boldsymbol{w}_0)_r} \left\{ \left( \sum_{\beta \in J_{\Delta \xi_1}} \hat{a}_{0,1,\beta} \frac{\partial^\beta}{\partial \boldsymbol{x}^\beta} \left( (\boldsymbol{w}_0)_r^\top \boldsymbol{y}_i \right)_+^p \right) \left( \sum_{\alpha \in J_1} \tilde{a}_\alpha \frac{\partial^\alpha}{\partial \boldsymbol{x}^\alpha} \left( (\boldsymbol{w}_0)_r^\top \tilde{\boldsymbol{y}}_j \right)_+^p \right) \right\} \right\|_2 \tag{169}$$

$$\leq \frac{1}{m} \sqrt{\frac{\nu}{n_o n_b}} \binom{d + \Delta \xi_1}{d} (d+1) a_{\max}^2 p^{\Delta \xi_1 + 2} \left\| (\boldsymbol{w}_0)_r \right\|_2^{2p-1} \tag{170}$$

$$\leq \frac{1}{m} \sqrt{\frac{\nu_{\max}}{n_o n_b}} \binom{d + |\xi|}{d} (d+1) a_{\max}^2 p^{|\xi|+2} \left\| (\boldsymbol{w}_0)_r \right\|_2^{2p-1}, \tag{171}$$

$$\frac{\partial}{\partial (\boldsymbol{w}_\ell)_r} \left( \boldsymbol{Q}_{0,L+1} \right)_{i,j} = \boldsymbol{0}, \ \forall \ell \neq 0, \tag{172}$$

$$\frac{\partial}{\partial (\boldsymbol{v}_\ell)_r} \boldsymbol{Q}_{i,j} = \boldsymbol{0}, \ \forall \ell. \tag{173}$$

For each $\ell \in [L]$, $\alpha_1, \alpha_2 \in I_{\xi_{\ell-1}}$, $\beta_1, \beta_2 \in I_{\Delta \xi_\ell}$, and $i_1, i_2 \in [n_o]$,

$$\left( \boldsymbol{Q}_{\ell,\ell} \right)_{((\alpha_1,\beta_1),i_1),((\alpha_2,\beta_2),i_2)} \tag{174}$$

$$= \frac{\nu_\ell}{m n_o} \sum_{r=1}^{m} \left( \left( \frac{\partial^{\beta_1}}{\partial \boldsymbol{x}^{\beta_1}} \left( (\boldsymbol{w}_{\ell-1})_r^\top \boldsymbol{y}_{i_1} \right)_+^p \right) \left( \frac{\partial^{\beta_2}}{\partial \boldsymbol{x}^{\beta_2}} \left( (\boldsymbol{w}_{\ell-1})_r^\top \boldsymbol{y}_{i_2} \right)_+^p \right) \boldsymbol{1}_{\{\alpha_1 = \alpha_2\}} \tag{175}$$

$$+ \left( (\boldsymbol{w}_\ell)_r^\top \boldsymbol{y}_{i_1} \right)_+^p \left( (\boldsymbol{w}_\ell)_r^\top \boldsymbol{y}_{i_2} \right)_+^p \boldsymbol{1}_{\{\alpha_1 + \beta_1 = \alpha_2 + \beta_2\}} \right), \tag{176}$$

$$\left\| \frac{\partial}{\partial (\boldsymbol{w}_{\ell-1})_r} \left( \boldsymbol{Q}_{\ell,\ell} \right)_{((\alpha_1,\beta_1),i_1),((\alpha_2,\beta_2),i_2)} \right\|_2 \tag{177}$$

$$= \boldsymbol{1}_{\{\alpha_1 = \alpha_2\}} \left\| \frac{\nu_\ell}{m n_o} \frac{\partial}{\partial (\boldsymbol{w}_{\ell-1})_r} \left\{ \left( \frac{\partial^{\beta_1}}{\partial \boldsymbol{x}^{\beta_1}} \left( (\boldsymbol{w}_{\ell-1})_r^\top \boldsymbol{y}_{i_1} \right)_+^p \right) \left( \frac{\partial^{\beta_2}}{\partial \boldsymbol{x}^{\beta_2}} \left( (\boldsymbol{w}_{\ell-1})_r^\top \boldsymbol{y}_{i_2} \right)_+^p \right) \right\} \right\|_2 \tag{178}$$

$$\leq \boldsymbol{1}_{\{\alpha_1 = \alpha_2\}} \frac{\nu_\ell}{m n_o} p^{2 \Delta \xi_\ell + 1} \left\| (\boldsymbol{w}_{\ell-1})_r \right\|_2^{2p-1} \tag{179}$$

$$\leq \boldsymbol{1}_{\{\alpha_1 = \alpha_2\}} \frac{\nu_{\max}}{m n_o} p^{2|\xi|+1} \left\| (\boldsymbol{w}_{\ell-1})_r \right\|_2^{2p-1}, \tag{180}$$

and

$$\left\| \frac{\partial}{\partial (\boldsymbol{w}_\ell)_r} \left( \boldsymbol{Q}_{\ell,\ell} \right)_{((\alpha_1,\beta_1),i_1),((\alpha_2,\beta_2),i_2)} \right\|_2 \tag{181}$$

$$= \boldsymbol{1}_{\{\alpha_1 + \beta_1 = \alpha_2 + \beta_2\}} \left\| \frac{\nu_\ell}{m n_o} \frac{\partial}{\partial (\boldsymbol{w}_\ell)_r} \left\{ \left( (\boldsymbol{w}_\ell)_r^\top \boldsymbol{y}_{i_1} \right)_+^p \left( (\boldsymbol{w}_\ell)_r^\top \boldsymbol{y}_{i_2} \right)_+^p \right\} \right\|_2 \tag{182}$$

$$\leq \boldsymbol{1}_{\{\alpha_1 + \beta_1 = \alpha_2 + \beta_2\}} \frac{\nu_\ell}{m n_o} p \left\| (\boldsymbol{w}_\ell)_r \right\|_2^{2p-1} \tag{183}$$

$$\leq \boldsymbol{1}_{\{\alpha_1 + \beta_1 = \alpha_2 + \beta_2\}} \frac{\nu_{\max}}{m n_o} p \left\| (\boldsymbol{w}_\ell)_r \right\|_2^{2p-1}, \tag{184}$$

$$\frac{\partial}{\partial (\boldsymbol{w}_{\ell'})_r} \left( \boldsymbol{Q}_{\ell,\ell} \right)_{i_1,i_2} = \boldsymbol{0}, \ \forall \ell' \neq \ell-1, \ell, \tag{185}$$

$$\frac{\partial}{\partial (\boldsymbol{v}_{\ell'})_r} \left( \boldsymbol{Q}_{\ell,\ell} \right)_{i_1,i_2} = \boldsymbol{0}, \ \forall \ell'. \tag{186}$$

Similarly, for each $\ell \in \{1, \cdots, L-1\}$, $\alpha_1 \in I_{\xi_{\ell-1}}$, $\alpha_2 \in I_{\xi_\ell}$, $\beta_1 \in I_{\Delta\xi_\ell}$, $\beta_2 \in I_{\Delta\xi_{\ell+1}}$ and $i_1, i_2 \in [n_o]$,

$$\left(\boldsymbol{Q}_{\ell,\ell+1}\right)_{((\alpha_1,\beta_1),i_1),((\alpha_2,\beta_2),i_2)} \tag{187}$$

$$= \frac{\sqrt{\nu_\ell \nu_{\ell+1}}}{mn_o} \sum_{r=1}^{m} \left(-\left((\boldsymbol{w}_\ell)_r^\top \boldsymbol{y}_{i_1}\right)_+^p\right) \left(\frac{\partial^{\beta_2}}{\partial \boldsymbol{x}^{\beta_2}} \left((\boldsymbol{w}_\ell)_r^\top \boldsymbol{y}_{i_2}\right)_+^p\right) \mathbf{1}_{\{\alpha_2=\alpha_1+\beta_1\}}, \tag{188}$$

$$\tag{189}$$

$$\left\| \frac{\partial}{\partial (\boldsymbol{w}_\ell)_r} \left(\boldsymbol{Q}_{\ell,\ell+1}\right)_{((\alpha_1,\beta_1),i_1),((\alpha_2,\beta_2),i_2)} \right\|_2 \tag{190}$$

$$\leq \mathbf{1}_{\{\alpha_2=\alpha_1+\beta_1\}} \frac{\sqrt{\nu_\ell \nu_{\ell+1}}}{mn_o} p^{\Delta\xi_{\ell+1}+1} \|(\boldsymbol{w}_\ell)_r\|_2^{2p-1} \tag{191}$$

$$\leq \mathbf{1}_{\{\alpha_2=\alpha_1+\beta_1\}} \frac{\nu_{\max}}{mn_o} p^{|\xi|+1} \|(\boldsymbol{w}_\ell)_r\|_2^{2p-1}, \tag{192}$$

$$\frac{\partial}{\partial (\boldsymbol{w}_{\ell'})_r} \left(\boldsymbol{Q}_{\ell,\ell+1}\right)_{((\alpha_1,\beta_1),i_1),((\alpha_2,\beta_2),i_2)} = \mathbf{0}, \ \forall \ell' \neq \ell, \tag{193}$$

$$\tag{194}$$

$$\frac{\partial}{\partial (\boldsymbol{v}_{\ell'})_r} \left(\boldsymbol{Q}_{\ell,\ell+1}\right)_{((\alpha_1,\beta_1),i_1),((\alpha_2,\beta_2),i_2)} = \mathbf{0}, \ \forall \ell'. \tag{195}$$

For each $\ell \in [L]$, $\alpha = 1$, $\beta \in I_{\Delta\xi_1}$, $i \in [n_o]$ and $j \in [n_b]$,

$$\left(\boldsymbol{Q}_{\ell,L+1}\right)_{((\alpha,\beta),i),j} \tag{196}$$

$$= \frac{1}{m}\sqrt{\frac{\nu_1 \nu}{n_o n_b}} \sum_{r=1}^{m} \left(\frac{\partial^\beta}{\partial \boldsymbol{x}^\beta} \left((\boldsymbol{w}_0)_r^\top \boldsymbol{y}_i\right)_+^p\right) \left(\sum_{\alpha \in J_1} \tilde{a}_\alpha \frac{\partial^\alpha}{\partial \boldsymbol{x}^\alpha} \left((\boldsymbol{w}_0)_r^\top \tilde{\boldsymbol{y}}_j\right)_+^p\right) \mathbf{1}_{\{\ell=1\}}, \tag{197}$$

$$\left\| \frac{\partial}{\partial (\boldsymbol{w}_0)_r} \left(\boldsymbol{Q}_{1,L+1}\right)_{((\alpha,\beta),i),j} \right\|_2 \tag{198}$$

$$= \left\| \frac{\sqrt{\nu_1 \nu}}{m\sqrt{n_o n_b}} \frac{\partial}{\partial (\boldsymbol{w}_0)_r} \left\{ \sum_{r=1}^{m} \left(\frac{\partial^\beta}{\partial \boldsymbol{x}^\beta} \left((\boldsymbol{w}_0)_r^\top \boldsymbol{y}_i\right)_+^p\right) \left(\sum_{\alpha \in J_1} \tilde{a}_\alpha \frac{\partial^\alpha}{\partial \boldsymbol{x}^\alpha} \left((\boldsymbol{w}_0)_r^\top \tilde{\boldsymbol{y}}_j\right)_+^p\right) \right\} \right\|_2 \tag{199}$$

$$\leq \frac{\sqrt{\nu_1 \nu}}{m\sqrt{n_o n_b}} a_{\max} (d+1) p^{\Delta\xi_1+2} \|(\boldsymbol{w}_0)_r\|_2^{2p-1} \tag{200}$$

$$\leq \frac{\nu_{\max}}{m\sqrt{n_o n_b}} a_{\max} (d+1) p^{|\xi|+2} \|(\boldsymbol{w}_0)_r\|_2^{2p-1}, \tag{201}$$

$$\frac{\partial}{\partial (\boldsymbol{w}_\ell)_r} \left(\boldsymbol{Q}_{1,L+1}\right)_{((\alpha,\beta),i),j} = \mathbf{0}, \ \forall \ell \neq 0, \tag{202}$$

$$\frac{\partial}{\partial (\boldsymbol{v}_\ell)_r} \left(\boldsymbol{Q}_{1,L+1}\right)_{((\alpha,\beta),i),j} = \mathbf{0}, \ \forall \ell. \tag{203}$$

For each $j_1, j_2 \in [n_b]$, the direct calculations lead to

$$\left(\boldsymbol{Q}_{L+1,L+1}\right)_{j_1,j_2} = \frac{\nu}{mn_b} \sum_{r=1}^{m} \left(\sum_{\alpha \in J_1} \tilde{a}_\alpha \frac{\partial^\alpha}{\partial \boldsymbol{x}^\alpha} \left((\boldsymbol{w}_0)_r^\top \tilde{\boldsymbol{y}}_{j_1}\right)_+^p\right) \left(\sum_{\alpha \in J_1} \tilde{a}_\alpha \frac{\partial^\alpha}{\partial \boldsymbol{x}^\alpha} \left((\boldsymbol{w}_0)_r^\top \tilde{\boldsymbol{y}}_{j_2}\right)_+^p\right),$$
$$\tag{204}$$

$$\left\| \frac{\partial}{\partial (\boldsymbol{w}_0)_r} \left(\boldsymbol{Q}_{L+1,L+1}\right)_{j_1,j_2} \right\|_2 \leq \frac{\nu}{mn_b} a_{\max}^2 (d+1)^2 p^3 \|(\boldsymbol{w}_0)_r\|_2^{2p-1}, \tag{205}$$

$$\frac{\partial}{\partial (\boldsymbol{w}_\ell)_r} \left(\boldsymbol{Q}_{L+1,L+1}\right)_{j_1,j_2} = \mathbf{0}, \ \forall \ell \neq 0, \tag{206}$$

$$\frac{\partial}{\partial (\boldsymbol{v}_\ell)_r} \left(\boldsymbol{Q}_{L+1,L+1}\right)_{j_1,j_2} = \mathbf{0}, \ \forall \ell. \tag{207}$$

**Hoeffding's inequalities for $Q_{\ell_1, \ell_2}$.** For each $i \in [n_o]$ and $j \in [n_b]$, we have

$$\left| \frac{\partial^\beta}{\partial \boldsymbol{x}^\beta} \left( (\boldsymbol{w}_\ell)_r^\top \boldsymbol{y}_i \right)_+^p \right| \leq p^{|\beta|} \left\| (\boldsymbol{w}_\ell)_r \right\|_2^p, \tag{208}$$

$$\left| \frac{\partial^\beta}{\partial \boldsymbol{x}^\beta} \left( (\boldsymbol{w}_{\ell-1})_r^\top \boldsymbol{y}_i \right)_+^p \right| \leq p^{|\beta|} \left\| (\boldsymbol{w}_{\ell-1})_r \right\|_2^p, \tag{209}$$

$$\left| \left( (\boldsymbol{w}_\ell)_r^\top \boldsymbol{y}_i \right)_+^p \right| \leq \left\| (\boldsymbol{w}_\ell)_r \right\|_2^p, \tag{210}$$

$$\left| \frac{\partial^\alpha}{\partial \boldsymbol{x}^\alpha} \left( (\boldsymbol{w}_0)_r^\top \tilde{\boldsymbol{y}}_j \right)_+^p \right| \leq p^{|\alpha|} \left\| (\boldsymbol{w}_0)_r \right\|_2^p. \tag{211}$$

Then, for each $i_1, i_2 \in [n_o]$,

$$\left| \sum_{\ell=0}^L \sum_{\alpha \in I_{\xi_\ell}} \left( \sum_{\beta \in J_{\Delta\xi_{\ell+1}}} \hat{a}_{\ell,\alpha,\beta} \frac{\partial^\beta}{\partial \boldsymbol{x}^\beta} \left( (\boldsymbol{w}_\ell)_r^\top \boldsymbol{y}_{i_1} \right)_+^p \right) \left( \sum_{\beta \in J_{\Delta\xi_{\ell+1}}} \hat{a}_{\ell,\alpha,\beta} \frac{\partial^\beta}{\partial \boldsymbol{x}^\beta} \left( (\boldsymbol{w}_\ell)_r^\top \boldsymbol{y}_{i_2} \right)_+^p \right) \right| \tag{212}$$

$$\leq \sum_{\ell=0}^L \sum_{\alpha \in I_{\xi_\ell}} \left( \sum_{\beta \in J_{\Delta\xi_{\ell+1}}} a_{\max} p^{|\beta|} \left\| (\boldsymbol{w}_\ell)_r \right\|_2^p \right) \left( \sum_{\beta \in J_{\Delta\xi_{\ell+1}}} a_{\max} p^{|\beta|} \left\| (\boldsymbol{w}_\ell)_r \right\|_2^p \right) \tag{213}$$

$$\leq \sum_{\ell=0}^L \sum_{\alpha \in I_{\xi_\ell}} \left| J_{\Delta\xi_{\ell+1}} \right|^2 a_{\max}^2 p^{2|\Delta\xi_{\ell+1}|} R^{2p} \tag{214}$$

$$\leq \sum_{\ell=0}^L |I_{\xi_\ell}| \binom{d+|\xi|}{d}^2 a_{\max}^2 p^{2|\xi|} R^{2p} \tag{215}$$

$$= \binom{d+k}{d} \binom{d+|\xi|}{d}^2 a_{\max}^2 p^{2|\xi|} R^{2p}. \tag{216}$$

By Hoeffding's inequality with

$$\left( \boldsymbol{Q}_{0,0} \right)_{i_1, i_2} \tag{217}$$

$$= \sum_{r=1}^m \sum_{\ell=0}^L \sum_{\alpha \in I_{\xi_\ell}} \left( \frac{\partial}{\partial (\boldsymbol{v}_\ell)_{r,\alpha}} \boldsymbol{s}_{i_1} \right) \left( \frac{\partial}{\partial (\boldsymbol{v}_\ell)_{r,\alpha}} \boldsymbol{s}_{i_2} \right) \tag{218}$$

$$= \frac{1}{m n_o} \sum_{\ell=0}^L \sum_{r=1}^m \sum_{\alpha \in I_{\xi_\ell}} \left( \sum_{\beta \in J_{\Delta\xi_{\ell+1}}} \hat{a}_{\ell,\alpha,\beta} \frac{\partial^\beta}{\partial \boldsymbol{x}^\beta} \left( (\boldsymbol{w}_\ell)_r^\top \boldsymbol{y}_{i_1} \right)_+^p \right) \left( \sum_{\beta \in J_{\Delta\xi_{\ell+1}}} \hat{a}_{\ell,\alpha,\beta} \frac{\partial^\beta}{\partial \boldsymbol{x}^\beta} \left( (\boldsymbol{w}_\ell)_r^\top \boldsymbol{y}_{i_2} \right)_+^p \right), \tag{219}$$

we have

$$\mathbb{P}\left[ \left| \left( \boldsymbol{Q}_{0,0} \right)_{i_1, i_2} - \mathbb{E}\left[ \left( \boldsymbol{Q}_{0,0} \right)_{i_1, i_2} \right] \right| > \varepsilon \right] < 2 \exp\left( -\frac{m n_o^2 \varepsilon^2}{2 \binom{d+k}{d}^2 \binom{d+|\xi|}{d}^4 a_{\max}^4 p^{4|\xi|} R^{4p}} \right), \tag{220}$$

for any $\varepsilon > 0$.

Similarly, for each $\ell \in [L]$, $\alpha \in I_{\xi_{\ell-1}}$, $\beta \in \Delta\xi_\ell$, and $i_1, i_2 \in [n_o]$,

$$\left| \left( \sum_{\beta' \in J_{\Delta\xi_\ell}} \hat{a}_{\ell-1,\alpha,\beta'} \frac{\partial^{\beta'}}{\partial \boldsymbol{x}^{\beta'}} \left( (\boldsymbol{w}_{\ell-1})_r^\top \boldsymbol{y}_{i_1} \right)_+^p \right) \left( \frac{\partial^\beta}{\partial \boldsymbol{x}^\beta} \left( (\boldsymbol{w}_{\ell-1})_r^\top \boldsymbol{y}_{i_2} \right)_+^p \right) \right. \tag{221}$$

$$\left. - \left( \sum_{\beta' \in J_{\Delta\xi_{\ell+1}}} \hat{a}_{\ell,\alpha,\beta'} \frac{\partial^{\beta'}}{\partial \boldsymbol{x}^{\beta'}} \left( (\boldsymbol{w}_\ell)_r^\top \boldsymbol{y}_{i_1} \right)_+^p \right) \left( (\boldsymbol{w}_\ell)_r^\top \boldsymbol{y}_{i_2} \right)_+^p \right| \tag{222}$$

$$\leq \binom{d + \Delta\xi_\ell}{\Delta\xi_\ell} a_{\max} p^{\Delta\xi_\ell} \left\| (\boldsymbol{w}_{\ell-1})_r \right\|_2^p \cdot p^{|\beta|} \left\| (\boldsymbol{w}_{\ell-1})_r \right\|_2^p + \binom{d + \Delta\xi_{\ell+1}}{\Delta\xi_{\ell+1}} a_{\max} p^{\Delta\xi_{\ell+1}} \left\| (\boldsymbol{w}_\ell)_r \right\|_2^p \left\| (\boldsymbol{w}_\ell)_r \right\|_2^p \tag{223}$$

$$\leq \binom{d + |\xi|}{d} \left( p^{|\xi|} + 1 \right) a_{\max} p^{|\xi|} R^{2p}. \tag{224}$$

By Hoeffding's inequality with

$$\left( \boldsymbol{Q}_{0,\ell} \right)_{i_1, ((\alpha,\beta),i_2)} \tag{225}$$

$$= \frac{\sqrt{\nu_\ell}}{m n_o} \sum_{r=1}^m \left( \left( \sum_{\beta' \in J_{\Delta\xi_\ell}} \hat{a}_{\ell-1,\alpha,\beta'} \frac{\partial^{\beta'}}{\partial \boldsymbol{x}^{\beta'}} \left( (\boldsymbol{w}_{\ell-1})_r^\top \boldsymbol{y}_{i_1} \right)_+^p \right) \left( \frac{\partial^\beta}{\partial \boldsymbol{x}^\beta} \left( (\boldsymbol{w}_{\ell-1})_r^\top \boldsymbol{y}_{i_2} \right)_+^p \right) \right. \tag{226}$$

$$\left. - \left( \sum_{\beta' \in J_{\Delta\xi_{\ell+1}}} \hat{a}_{\ell,\alpha,\beta'} \frac{\partial^{\beta'}}{\partial \boldsymbol{x}^{\beta'}} \left( (\boldsymbol{w}_\ell)_r^\top \boldsymbol{y}_{i_1} \right)_+^p \right) \left( (\boldsymbol{w}_\ell)_r^\top \boldsymbol{y}_{i_2} \right)_+^p \right), \tag{227}$$

we have

$$\mathbb{P}\left[ \left| \left( \boldsymbol{Q}_{0,\ell} \right)_{i_1,((\alpha,\beta),i_2)} - \mathbb{E}\left[ \left( \boldsymbol{Q}_{0,\ell} \right)_{i_1,((\alpha,\beta),i_2)} \right] \right| > \varepsilon \right] < 2\exp\left( - \frac{m n_o^2 \varepsilon^2}{2\nu_\ell \binom{d+|\xi|}{d}^2 \left( p^{|\xi|} + 1 \right)^2 a_{\max}^2 p^{2|\xi|} R^{4p}} \right), \tag{228}$$

for any $\varepsilon > 0$.

For each $i \in [n_o]$ and $j \in [n_b]$,

$$\left| \left( \sum_{\beta \in J_{\Delta\xi_1}} \hat{a}_{0,1,\beta} \frac{\partial^\beta}{\partial \boldsymbol{x}^\beta} \left( (\boldsymbol{w}_0)_r^\top \boldsymbol{y}_i \right)_+^p \right) \left( \sum_{\alpha \in J_1} \tilde{a}_\alpha \frac{\partial^\alpha}{\partial \boldsymbol{x}^\alpha} \left( (\boldsymbol{w}_0)_r^\top \tilde{\boldsymbol{y}}_j \right)_+^p \right) \right| \tag{229}$$

$$\leq \binom{d + \Delta\xi_1}{\Delta\xi_1} a_{\max} p^{\Delta\xi_1} \left\| (\boldsymbol{w}_0)_r \right\|_2^p \binom{d+1}{1} a_{\max} p^{1} \left\| (\boldsymbol{w}_0)_r \right\|_2^p \tag{230}$$

$$\leq (d+1) \binom{d+|\xi|}{d} a_{\max}^2 p^{|\xi|+1} R^{2p}. \tag{231}$$

By Hoeffding's inequality with

$$\left( \boldsymbol{Q}_{0,L+1} \right)_{i,j} = \frac{1}{m} \sqrt{\frac{\nu}{n_o n_b}} \sum_{r=1}^m \left( \sum_{\beta \in J_{\Delta\xi_1}} \hat{a}_{0,1,\beta} \frac{\partial^\beta}{\partial \boldsymbol{x}^\beta} \left( (\boldsymbol{w}_0)_r^\top \boldsymbol{y}_i \right)_+^p \right) \left( \sum_{\alpha \in J_1} \tilde{a}_\alpha \frac{\partial^\alpha}{\partial \boldsymbol{x}^\alpha} \left( (\boldsymbol{w}_0)_r^\top \tilde{\boldsymbol{y}}_j \right)_+^p \right), \tag{232}$$

we have

$$\mathbb{P}\left[ \left| \left( \boldsymbol{Q}_{0,L+1} \right)_{i,j} - \mathbb{E}\left[ \left( \boldsymbol{Q}_{0,L+1} \right)_{i,j} \right] \right| > \varepsilon \right] < 2\exp\left( - \frac{m n_o n_b \varepsilon^2}{2\nu (d+1)^2 \binom{d+|\xi|}{d}^2 a_{\max}^4 p^{2|\xi|+2} R^{4p}} \right), \tag{233}$$

for any $\varepsilon > 0$.

For each $\ell \in [L]$, $\alpha_1, \alpha_2 \in I_{\xi_{\ell-1}}$, $\beta_1, \beta_2 \in J_{\Delta\xi_\ell}$, and $i_1, i_2 \in [n_o]$,

$$\left| \left( \frac{\partial^{\beta_1}}{\partial \boldsymbol{x}^{\beta_1}} \left( (\boldsymbol{w}_{\ell-1})_r^\top \boldsymbol{y}_{i_1} \right)_+^p \right) \left( \frac{\partial^{\beta_2}}{\partial \boldsymbol{x}^{\beta_2}} \left( (\boldsymbol{w}_{\ell-1})_r^\top \boldsymbol{y}_{i_2} \right)_+^p \right) \mathbf{1}_{\{\alpha_1 = \alpha_2\}} \right. \tag{234}$$

$$\left. + \left( (\boldsymbol{w}_\ell)_r^\top \boldsymbol{y}_{i_1} \right)_+^p \left( (\boldsymbol{w}_\ell)_r^\top \boldsymbol{y}_{i_2} \right)_+^p \mathbf{1}_{\{\alpha_1+\beta_1 = \alpha_2+\beta_2\}} \right| \tag{235}$$

$$\leq p^{|\beta_1|+|\beta_2|} R^{2p} \mathbf{1}_{\{\alpha_1 = \alpha_2\}} + R^{2p} \mathbf{1}_{\{\alpha_1+\beta_1 = \alpha_2+\beta_2\}} \tag{236}$$

$$\leq \left( p^{2|\xi|} + 1 \right) R^{2p}. \tag{237}$$

By Hoeffding's inequality with

$$\left( \boldsymbol{Q}_{\ell,\ell} \right)_{((\alpha_1,\beta_1),i_1),((\alpha_2,\beta_2),i_2)} \tag{238}$$

$$= \frac{\nu_\ell}{mn_o} \sum_{r=1}^m \left( \left( \frac{\partial^{\beta_1}}{\partial \boldsymbol{x}^{\beta_1}} \left( (\boldsymbol{w}_{\ell-1})_r^\top \boldsymbol{y}_{i_1} \right)_+^p \right) \left( \frac{\partial^{\beta_2}}{\partial \boldsymbol{x}^{\beta_2}} \left( (\boldsymbol{w}_{\ell-1})_r^\top \boldsymbol{y}_{i_2} \right)_+^p \right) \mathbf{1}_{\{\alpha_1=\alpha_2\}} \right. \tag{239}$$

$$\left. + \left( (\boldsymbol{w}_\ell)_r^\top \boldsymbol{y}_{i_1} \right)_+^p \left( (\boldsymbol{w}_\ell)_r^\top \boldsymbol{y}_{i_2} \right)_+^p \mathbf{1}_{\{\alpha_1+\beta_1=\alpha_2+\beta_2\}} \right), \tag{240}$$

we have

$$\mathbb{P}\left[ \left| \left( \boldsymbol{Q}_{\ell,\ell} \right)_{((\alpha_1,\beta_1),i_1),((\alpha_2,\beta_2),i_2)} - \mathbb{E}\left[ \left( \boldsymbol{Q}_{\ell,\ell} \right)_{((\alpha_1,\beta_1),i_1),((\alpha_2,\beta_2),i_2)} \right] \right| > \varepsilon \right] < 2\exp\left( -\frac{mn_o^2\varepsilon^2}{2\nu_\ell^2 \left( p^{2|\xi|}+1 \right)^2 R^{4p}} \right), \tag{241}$$

for any $\varepsilon > 0$.

For each $\ell \in [L]$, $\alpha_1 \in I_{\xi_{\ell-1}}$, $\alpha_2 \in I_{\xi_\ell}$, $\beta_1 \in I_{\Delta\xi_\ell}$, $\beta_2 \in I_{\Delta\xi_{\ell+1}}$, and $i_1, i_2 \in [n_o]$,

$$\left| \left( (\boldsymbol{w}_\ell)_r^\top \boldsymbol{y}_{i_1} \right)_+^p \left( \frac{\partial^{\beta_2}}{\partial \boldsymbol{x}^{\beta_2}} \left( (\boldsymbol{w}_\ell)_r^\top \boldsymbol{y}_{i_2} \right)_+^p \right) \mathbf{1}_{\{\alpha_2=\alpha_1+\beta_2\}} \right| \le p^{|\xi|} R^{2p}. \tag{242}$$

By Hoeffding's inequality with

$$\left( \boldsymbol{Q}_{\ell,\ell+1} \right)_{((\alpha_1,\beta_1),i_1),((\alpha_2,\beta_2),i_2)} \tag{243}$$

$$= \frac{\sqrt{\nu_\ell \nu_{\ell+1}}}{mn_o} \sum_{r=1}^m \left( -\left( (\boldsymbol{w}_\ell)_r^\top \boldsymbol{y}_{i_1} \right)_+^p \right) \left( \frac{\partial^{\beta_2}}{\partial \boldsymbol{x}^{\beta_2}} \left( (\boldsymbol{w}_\ell)_r^\top \boldsymbol{y}_{i_2} \right)_+^p \right) \mathbf{1}_{\{\alpha_2=\alpha_1+\beta_2\}}, \tag{244}$$

we have

$$\mathbb{P}\left[ \left| \left( \boldsymbol{Q}_{\ell,\ell+1} \right)_{((\alpha_1,\beta_1),i_1),((\alpha_2,\beta_2),i_2)} - \mathbb{E}\left[ \left( \boldsymbol{Q}_{\ell,\ell+1} \right)_{((\alpha_1,\beta_1),i_1),((\alpha_2,\beta_2),i_2)} \right] \right| > \varepsilon \right] < 2\exp\left( -\frac{mn_o^2\varepsilon^2}{2\nu_\ell\nu_{\ell+1} p^{2|\xi|} R^{4p}} \right), \tag{245}$$

for any $\varepsilon > 0$.

For each $\alpha \in I_{\xi_{\ell-1}}$, $\beta \in J_{\Delta\xi_\ell}$, $i \in [n_o]$, and $j \in [n_b]$,

$$\left| \frac{\partial^\beta}{\partial \boldsymbol{x}^\beta} \left( (\boldsymbol{w}_0)_r^\top \boldsymbol{y}_i \right)_+^p \left( \sum_{\alpha' \in J_1} \tilde{a}_{\alpha'} \frac{\partial^{\alpha'}}{\partial \boldsymbol{x}^{\alpha'}} \left( (\boldsymbol{w}_0)_r^\top \tilde{\boldsymbol{y}}_j \right)_+^p \right) \right| \tag{246}$$

$$\le p^{|\beta|} R^p \sum_{\alpha' \in J_1} a_{\max} p^{|\alpha'|} R^p \tag{247}$$

$$\le (d+1) a_{\max} p^{|\xi|+1} R^{2p}. \tag{248}$$

By Hoeffding's inequality with

$$\left( \boldsymbol{Q}_{1,L+1} \right)_{((\alpha,\beta),i),j} \tag{249}$$

$$= \frac{1}{m} \sqrt{\frac{\nu_1 \nu}{n_o n_b}} \sum_{r=1}^m \left( \frac{\partial^\beta}{\partial \boldsymbol{x}^\beta} \left( (\boldsymbol{w}_0)_r^\top \boldsymbol{y}_i \right)_+^p \right) \left( \sum_{\alpha' \in J_1} \tilde{a}_{\alpha'} \frac{\partial^{\alpha'}}{\partial \boldsymbol{x}^{\alpha'}} \left( (\boldsymbol{w}_0)_r^\top \tilde{\boldsymbol{y}}_j \right)_+^p \right), \tag{250}$$

we have

$$\mathbb{P}\left[ \left| \left( \boldsymbol{Q}_{1,L+1} \right)_{((\alpha,\beta),i),j} - \mathbb{E}\left[ \left( \boldsymbol{Q}_{1,L+1} \right)_{((\alpha,\beta),i),j} \right] \right| > \varepsilon \right] < 2\exp\left( -\frac{mn_o n_b \varepsilon^2}{2\nu_1 \nu (d+1)^2 a_{\max}^2 p^{2|\xi|+2} R^{4p}} \right), \tag{251}$$

for any $\varepsilon > 0$.

For $j_1, j_2 \in [n_b]$,

$$\left( \sum_{\alpha \in J_1} \tilde{a}_\alpha \frac{\partial^\alpha}{\partial \boldsymbol{x}^\alpha} \left( (\boldsymbol{w}_0)_r^\top \tilde{\boldsymbol{y}}_{j_1} \right)_+^p \right) \left( \sum_{\alpha \in J_1} \tilde{a}_\alpha \frac{\partial^\alpha}{\partial \boldsymbol{x}^\alpha} \left( (\boldsymbol{w}_0)_r^\top \tilde{\boldsymbol{y}}_{j_2} \right)_+^p \right) \le (d+1)^2 a_{\max}^2 p^2 R^{2p}. \tag{252}$$

By Hoeffding's inequality with

$$\left( \boldsymbol{Q}_{L+1,L+1} \right)_{j_1,j_2} = \frac{\nu}{mn_b} \sum_{r=1}^m \left( \sum_{\alpha \in J_1} \tilde{a}_\alpha \frac{\partial^\alpha}{\partial \boldsymbol{x}^\alpha} \left( (\boldsymbol{w}_0)_r^\top \tilde{\boldsymbol{y}}_{j_1} \right)_+^p \right) \left( \sum_{\alpha \in J_1} \tilde{a}_\alpha \frac{\partial^\alpha}{\partial \boldsymbol{x}^\alpha} \left( (\boldsymbol{w}_0)_r^\top \tilde{\boldsymbol{y}}_{j_2} \right)_+^p \right), \tag{253}$$

we have

$$\mathbb{P}\left[\left|\left(\boldsymbol{Q}_{L+1,L+1}\right)_{j_1,j_2} - \mathbb{E}\left[\left(\boldsymbol{Q}_{L+1,L+1}\right)_{j_1,j_2}\right]\right| > \varepsilon\right] < 2\exp\left(-\frac{mn_b^2\varepsilon^2}{2\nu^2\left(d+1\right)^4 a_{\max}^4 p^4 R^{4p}}\right),$$

(254)

for any $\varepsilon > 0$.

## C  Proofs

This section proves the main theorem with some lemmas and propositions. Similar to the [25], we first prove that the Gram matrix $\widehat{\boldsymbol{G}}_{\boldsymbol{v}}^{\infty}$ is independent of $m$ and strictly positive definite, in Proposition C.3. Second, as $\widehat{\boldsymbol{G}}_{\boldsymbol{v}}\left(\boldsymbol{w}\left(0\right),\boldsymbol{v}\left(0\right)\right)$ is the sample mean, Proposition C.4 shows that $\widehat{\boldsymbol{G}}_{\boldsymbol{v}}\left(\boldsymbol{w}\left(0\right),\boldsymbol{v}\left(0\right)\right)$ is close to its expected value $\widehat{\boldsymbol{G}}_{\boldsymbol{v}}^{\infty}$ with a high probability, if $m$ is large enough. This implies that $\widehat{\boldsymbol{G}}_{\boldsymbol{v}}\left(\boldsymbol{w}\left(0\right),\boldsymbol{v}\left(0\right)\right)$ is strictly positive definite with high probability. Then, we show that the initial loss is bounded with a high probability if $m$ is large. This is because $\boldsymbol{v}\left(0\right)$ has zero mean, which hinders the output of $m$ variables to diverge as $m$ increases. Finally, we prove that the Gram matrix $\widehat{\boldsymbol{G}}_{\boldsymbol{v}}\left(\boldsymbol{w}\left(t\right),\boldsymbol{v}\left(t\right)\right)$ remains within a small neighborhood of the initial point, where it maintains its strict positive definiteness. The gradient flow converges within the neighborhood, as the smallest eigenvalue of $\widehat{\boldsymbol{G}}_{\boldsymbol{v}}$ is distinct from zero, resulting in a rapid reduction of the loss.

**Lemma C.1.** *Let $d, n \in \mathbb{N}$ and $\boldsymbol{y}_1, \ldots, \boldsymbol{y}_n \in \mathbb{R}^d$ be vectors such that $\boldsymbol{y}_i \parallel \boldsymbol{y}_j$ if and only if $i = j$. Then, for each $i \in [n]$, there exists $\boldsymbol{w}_i$ such that $\boldsymbol{w}_i^\top \boldsymbol{y}_j = 0$ if and only if $i = j$.*

*Proof.* Let $\boldsymbol{y}_i^\perp = \left\{\boldsymbol{w} \in \mathbb{R}^d : \boldsymbol{w}^\top \boldsymbol{y}_i = 0\right\}$. Since $\boldsymbol{y}_i \parallel \boldsymbol{y}_j$ for $i \neq j$, $\boldsymbol{y}_i \cap \boldsymbol{y}_j$ is nowhere dense in $\boldsymbol{y}_i^\perp$. Hence, finite union of $\cup_{j \neq i} \boldsymbol{y}_j^\perp$ is nowhere dense in $\boldsymbol{y}_i^\perp$, and in particular, there exists $\boldsymbol{w}_i \in \boldsymbol{y}_i^\perp$ such that $\boldsymbol{w}_i \notin \cup_{j \neq i} \boldsymbol{y}_j^\perp$. $\qquad\square$

**Lemma C.2.** *Let $n, d \in \mathbb{N}$, and $\psi : \mathbb{R}^d \to \mathbb{R}$ be a homogeneous polynomial of degree $n$. For any $i \in \mathbb{N}$ and nonzero $\boldsymbol{y} = (y_1, \ldots, y_d) \in \mathbb{R}^d$, if a function $\psi\left(\boldsymbol{w}\right)\left(\boldsymbol{w}^\top \boldsymbol{y}\right)_+^i$ of $\boldsymbol{w} = (w_1, \ldots, w_d) \in \mathbb{R}^d$ is $(n + i)$-times continuously differentiable at some $\boldsymbol{w}^* \in \mathbb{R}^d$ with $\left(\boldsymbol{w}^*\right)^\top \boldsymbol{y} = 0$, then $\psi \equiv 0$.*

*Proof.* Without loss of the generality, we may assume the $y_1 \neq 0$. Write $\psi\left(\boldsymbol{w}\right) = \sum_{\alpha \in I_n} a_\alpha \boldsymbol{w}^\alpha$ for coefficients $a_\alpha \in \mathbb{R}$ and define $\varphi\left(\boldsymbol{w}\right) = \psi\left(\boldsymbol{w}\right)\left(\boldsymbol{w}^\top \boldsymbol{y}\right)_+^i$. For any $\alpha \in I_n$, the assumption on differentiability implies that $\frac{\partial^i}{\partial w_1^i}\frac{\partial^n}{\partial \boldsymbol{w}^\alpha}\varphi$ is continuous. Indeed, we have $\varphi\left(\boldsymbol{w}\right) = 0$ for $\boldsymbol{w}^\top \boldsymbol{y} \leq 0$, and

$$\frac{\partial^i}{\partial w_1^i}\frac{\partial^n}{\partial \boldsymbol{w}^\alpha}\varphi\left(\boldsymbol{w}\right) = \alpha! a_\alpha y_1^i,$$

(255)

for $\boldsymbol{w}^\top \boldsymbol{y} > 0$. The continuity induces $a_\alpha = 0$ and thus $\psi \equiv 0$ for all $\boldsymbol{w} \in \mathbb{R}^d$. $\qquad\square$

**Proposition C.3.** *$\widehat{\boldsymbol{G}}_{\boldsymbol{v}}^{\infty} = \mathbb{E}_{\boldsymbol{w},\boldsymbol{v}}\left[\widehat{\boldsymbol{G}}_{\boldsymbol{v}}\left(\boldsymbol{w},\boldsymbol{v}\right)\right]$ is strictly positive definite and independent of $m$.*

*Proof.* For $i_1, i_2 \in [n_o]$,

$$\left(\boldsymbol{Q}_{0,0}\right)_{i_1,i_2}$$

(256)

$$= \frac{1}{m}\sum_{r=1}^{m}\frac{1}{n_o}\sum_{\ell=0}^{L}\sum_{\alpha \in I_{\xi_\ell}}\left(\sum_{\beta \in J_{\Delta\xi_{\ell+1}}}\hat{a}_{\ell,\alpha,\beta}\frac{\partial^\beta}{\partial \boldsymbol{x}^\beta}\left(\left(\boldsymbol{w}_\ell\right)_r^\top \boldsymbol{y}_{i_1}\right)_+^p\right)\left(\sum_{\beta \in J_{\Delta\xi_{\ell+1}}}\hat{a}_{\ell,\alpha,\beta}\frac{\partial^\beta}{\partial \boldsymbol{x}^\beta}\left(\left(\boldsymbol{w}_\ell\right)_r^\top \boldsymbol{y}_{i_2}\right)_+^p\right),$$

(257)

and $m$ random variables

$$\frac{1}{n_o}\sum_{\ell=0}^{L}\sum_{\alpha \in I_{\xi_\ell}}\left(\sum_{\beta \in J_{\Delta\xi_{\ell+1}}}\hat{a}_{\ell,\alpha,\beta}\frac{\partial^\beta}{\partial \boldsymbol{x}^\beta}\left(\left(\boldsymbol{w}_\ell\right)_r^\top \boldsymbol{y}_{i_1}\right)_+^p\right)\left(\sum_{\beta \in J_{\Delta\xi_{\ell+1}}}\hat{a}_{\ell,\alpha,\beta}\frac{\partial^\beta}{\partial \boldsymbol{x}^\beta}\left(\left(\boldsymbol{w}_\ell\right)_r^\top \boldsymbol{y}_{i_2}\right)_+^p\right),$$

(258)

for $r \in [m]$, are independent and identically distributed. Therefore, the expected value of each component of $\boldsymbol{Q}_{0,0}$ is independent of $m$, as the expected value of the sample mean of independent and identically distributed random variables is independent of the number of samples. Similarly, each component of $\widehat{\boldsymbol{G}}_{\boldsymbol{v}}\left(\boldsymbol{w}, \boldsymbol{v}\right)$ is the sample mean of $m$ independent and identically distributed random variables, and its expected value $\widehat{\boldsymbol{G}}_{\boldsymbol{v}}^{\infty}$ is independent of $m$. To show that $\widehat{\boldsymbol{G}}_{\boldsymbol{v}}^{\infty}$ is strictly positive definite, assume that there exists $\boldsymbol{z}_0 = \left[(\boldsymbol{z}_0)_1 \quad \cdots \quad (\boldsymbol{z}_0)_{n_o}\right] \in \mathbb{R}^{n_o}$, $\tilde{\boldsymbol{z}} = \left[\tilde{z}_1 \quad \cdots \quad \tilde{z}_{n_b}\right]^{\top} \in \mathbb{R}^{n_b}$, and

$$\boldsymbol{z}_{\ell} = \left[(\boldsymbol{z}_{\ell})_{(\alpha,\beta),i}\right]_{\left(\alpha \in I_{\xi_{\ell}}, \beta \in I_{\Delta\xi_{\ell}}, i \in [n_o]\right)} \in \mathbb{R}^{|I_{\xi_{\ell-1}}| \cdot |I_{\Delta\xi_{\ell}}| n_o} \tag{259}$$

for $\ell \in [L]$, such that $\boldsymbol{z} = \left[\boldsymbol{z}_0^{\top} \quad \boldsymbol{z}_1^{\top} \quad \cdots \quad \boldsymbol{z}_L^{\top} \quad \tilde{\boldsymbol{z}}^{\top}\right]^{\top} \in \mathbb{R}^{n_o + \sum_{\ell=1}^{L}\left(|I_{\xi_{\ell-1}}| \cdot |I_{\Delta\xi_{\ell}}| n_o\right) + n_b}$ satisfies $\widehat{\boldsymbol{G}}_{\boldsymbol{v}}^{\infty} \boldsymbol{z} = \boldsymbol{0}$. Since $\widehat{\boldsymbol{G}}_{\boldsymbol{v}}^{\infty} \boldsymbol{z} = \mathbb{E}_{\boldsymbol{w},\boldsymbol{v}}\left[\hat{\boldsymbol{D}}_{\boldsymbol{v}}^{\top} \hat{\boldsymbol{D}}_{\boldsymbol{v}}\right] \boldsymbol{z} = 0$ if and only if $\mathbb{E}_{\boldsymbol{w},\boldsymbol{v}}\left[\left\|\hat{\boldsymbol{D}}_{\boldsymbol{v}} \boldsymbol{z}\right\|_2^2\right] = \boldsymbol{0}$, we show that $\mathbb{E}_{\boldsymbol{w},\boldsymbol{v}}\left[\left\|\hat{\boldsymbol{D}}_{\boldsymbol{v}} \boldsymbol{z}\right\|_2^2\right] = \boldsymbol{0}$ implies $\boldsymbol{z} = \boldsymbol{0}$ and thereby 0 is not an eigenvalue of $\widehat{\boldsymbol{G}}_{\boldsymbol{v}}^{\infty}$.

Now assume that there exists $\boldsymbol{z} \in \mathbb{R}^d$ such that $\hat{\boldsymbol{D}}_{\boldsymbol{v}} \boldsymbol{z} = \boldsymbol{0}$ for almost all $\boldsymbol{w}$ and $\boldsymbol{v}$. Since $\hat{\boldsymbol{D}}_{\boldsymbol{v}}$ is continuous with respect to $\boldsymbol{w}$ and $\boldsymbol{v}$, we have $\hat{\boldsymbol{D}}_{\boldsymbol{v}} \boldsymbol{z} = \boldsymbol{0}$ for all $\boldsymbol{w}$ and $\boldsymbol{v}$. Hence, the function $\hat{\boldsymbol{D}}_{\boldsymbol{v}} \boldsymbol{z}$ of $\boldsymbol{w}$ and $\boldsymbol{v}$, which is identically zero, is smooth with respect to $\boldsymbol{w}$ and $\boldsymbol{v}$.

In this proof, we denote an index for a component of $\hat{\boldsymbol{D}}_{\boldsymbol{v}} \boldsymbol{z}$ by $\ell \in [L]$, $r \in [m]$ and $\alpha \in I_{\xi_{\ell}}$, because each row of $\hat{\boldsymbol{D}}_{\boldsymbol{v}}$ corresponds to the partial derivatives with respect to $(\boldsymbol{v}_{\ell})_{r,\alpha}$ for $\ell \in [L]$, $r \in [m]$ and $\alpha \in I_{\xi_{\ell}}$. For instance, $\left(\hat{\boldsymbol{D}}_{\boldsymbol{v}} \boldsymbol{z}\right)_{0,1,\boldsymbol{0}}$ is the first component of $\left(\hat{\boldsymbol{D}}_{\boldsymbol{v}} \boldsymbol{z}\right)$. First, for $\ell = L$ and any $r \in [m]$ and $\alpha \in I_{\xi_L}$, we have

$$\left(\hat{\boldsymbol{D}}_{\boldsymbol{v}} \boldsymbol{z}\right)_{L,r,\alpha} \tag{260}$$

$$= \sum_{i=1}^{n_o} \left(\frac{\partial}{\partial (\boldsymbol{v}_L)_{r,\alpha}} s_i\right) (\boldsymbol{z}_0)_i + \sum_{i=1}^{n_o} \sum_{\alpha' \in I_{\xi_{L-1}}} \sum_{\beta \in I_{\Delta\xi_L}} \left(\frac{\partial}{\partial (\boldsymbol{v}_L)_{r,\alpha}} (\boldsymbol{\mu}_L)_{(\alpha',\beta),i}\right) (\boldsymbol{z}_L)_{(\alpha',\beta),i} \tag{261}$$

$$= \sqrt{\frac{1}{mn_o}} \sum_{i=1}^{n_o} \left(\sum_{\beta \in J_{\Delta\xi_{L+1}}} \hat{a}_{L,\alpha,\beta} \frac{\partial^{\beta}}{\partial \boldsymbol{x}^{\beta}} \left(\left(\boldsymbol{w}_L\right)_r^{\top} \boldsymbol{y}_i\right)_+^p\right) (\boldsymbol{z}_0)_i \tag{262}$$

$$- \sqrt{\frac{\nu_L}{mn_o}} \sum_{i=1}^{n_o} \sum_{\alpha' \in I_{\xi_{L-1}}} \sum_{\beta \in I_{\Delta\xi_L}} \left(\left(\boldsymbol{w}_L\right)_r^{\top} \boldsymbol{y}_i\right)_+^p (\boldsymbol{z}_L)_{(\alpha',\beta),i} \mathbf{1}_{\{\alpha = \alpha' + \beta\}}. \tag{263}$$

Fix $i \in [n_o]$, and by Lemma C.1, there exists $(\boldsymbol{w}_L^*)_r \in \mathbb{R}^d$ such that $(\boldsymbol{w}_L^*)_r^{\top} \boldsymbol{y}_i = 0$, $(\boldsymbol{w}_L^*)_r^{\top} \boldsymbol{y}_{i'} \neq 0$ for $i \neq i' \in [n_o]$ and $(\boldsymbol{w}_L^*)_r^{\top} \tilde{\boldsymbol{y}}_j \neq 0$ for $j \in [n_b]$. As a function of $(\boldsymbol{w}_L)_r$, $\hat{\boldsymbol{D}}_{\boldsymbol{v}} \boldsymbol{z} = \boldsymbol{0}$ implies that

$$\left(\sum_{\beta \in J_{\Delta\xi_{L+1}}} \hat{a}_{L,\alpha,\beta} \frac{\partial^{\beta}}{\partial \boldsymbol{x}^{\beta}} \left(\left(\boldsymbol{w}_L\right)_r^{\top} \boldsymbol{y}_i\right)_+^p\right) (\boldsymbol{z}_0)_i \tag{264}$$

$$- \sqrt{\nu_L} \sum_{\alpha' \in I_{\xi_{L-1}}} \sum_{\beta \in I_{\Delta\xi_L}} \left(\left(\boldsymbol{w}_L\right)_r^{\top} \boldsymbol{y}_i\right)_+^p (\boldsymbol{z}_L)_{(\alpha',\beta),i} \mathbf{1}_{\{\alpha = \alpha' + \beta\}} \tag{265}$$

$$= -\sum_{i' \neq i} \left(\sum_{\beta \in J_{\Delta\xi_{L+1}}} \hat{a}_{L,\alpha,\beta} \frac{\partial^{\beta}}{\partial \boldsymbol{x}^{\beta}} \left(\left(\boldsymbol{w}_L\right)_r^{\top} \boldsymbol{y}_i\right)_+^p\right) (\boldsymbol{z}_0)_i \tag{266}$$

$$+ \sqrt{\nu_L} \sum_{i' \neq i} \sum_{\alpha' \in I_{\xi_{L-1}}} \sum_{\beta \in I_{\Delta\xi_L}} \left(\left(\boldsymbol{w}_L\right)_r^{\top} \boldsymbol{y}_i\right)_+^p (\boldsymbol{z}_L)_{(\alpha',\beta),i} \mathbf{1}_{\{\alpha = \alpha' + \beta\}}. \tag{267}$$

is smooth at $(\boldsymbol{w}_L)_r = (\boldsymbol{w}_L^*)_r$. Note that the coefficient of $\left((\boldsymbol{w}_L)_r^\top \boldsymbol{y}_i\right)_+^{p-\Delta\xi_{L+1}}$ in (266) is given by

$$\sum_{\beta\in I_{\Delta\xi_{L+1}}} \hat{a}_{L,\alpha,\beta}\left(\boldsymbol{x}_i\right)(\boldsymbol{w}_L)_{r,1}^{\beta_1}\cdots(\boldsymbol{w}_L)_{r,d}^{\beta_d}\left(\boldsymbol{z}_0\right)_i, \tag{268}$$

which is a homogeneous polynomial of $\boldsymbol{w}_L$. Together with the fact that at least one of $\hat{a}_{L,\alpha,\beta}\left(\boldsymbol{x}_i\right)$ is nonzero, Lemma C.2 implies $(\boldsymbol{z}_0)_i = 0$. Similarly, for any $r \in [m]$,

$$\left(\hat{\boldsymbol{D}}_{\boldsymbol{v}}\boldsymbol{z}\right)_{0,r,\boldsymbol{0}} = \sum_{j=1}^{n_b}\left(\frac{\partial}{\partial(\boldsymbol{v}_0)_{r,\boldsymbol{0}}}h_j\right)\tilde{z}_j \tag{269}$$

$$= \sqrt{\frac{\nu}{mn_b}}\sum_{j=1}^{n_b}\sum_{\alpha\in J_1}\tilde{a}_\alpha\left(\tilde{\boldsymbol{x}}_j\right)\frac{\partial^\alpha}{\partial\boldsymbol{x}^\alpha}\left((\boldsymbol{w}_0)_r^\top\tilde{\boldsymbol{y}}_j\right)_+^p\tilde{z}_j \tag{270}$$

$$= \sqrt{\frac{\nu}{mn_b}}\sum_{j=1}^{n_b}\sum_{\alpha\in J_1}\tilde{a}_\alpha\left(\tilde{\boldsymbol{x}}_j\right)\tilde{z}_j(\boldsymbol{w}_0)_{r,1}^{\alpha_1}\cdots(\boldsymbol{w}_0)_{r,d}^{\alpha_d}\left((\boldsymbol{w}_0)_r^\top\tilde{\boldsymbol{y}}_j\right)_+^{p-|\alpha|} \tag{271}$$

deduces $\tilde{z}_j = 0$ with $\boldsymbol{w}_0^*$, instead of $\boldsymbol{w}_L^*$, such that $(\boldsymbol{w}_0^*)^\top\tilde{\boldsymbol{y}}_j = 0$, $(\boldsymbol{w}_0^*)^\top\tilde{\boldsymbol{y}}_{j'} \neq= 0$ for $j' \neq j$, and $(\boldsymbol{w}_0^*)^\top\boldsymbol{y}_i \neq 0$ for $i \in [n_o]$. Now suppose that we have $\tilde{\boldsymbol{z}} = 0$ and $\boldsymbol{z}_{\ell'} = 0$ for $\ell' = 0, 1, \ldots, \ell - 1$ for some $\ell \geq 1$. Then, for any $r \in [m]$ and $\alpha \in I_{\xi_{\ell-1}}$,

$$\left(\hat{\boldsymbol{D}}_{\boldsymbol{v}}\boldsymbol{z}\right)_{\ell-1,r,\alpha} \tag{272}$$

$$= \sum_{i=1}^{n_o}\left(\frac{\partial}{\partial(\boldsymbol{v}_{\ell-1})_{r,\alpha}}s_i\right)(\boldsymbol{z}_0)_i + \sum_{i=1}^{n_o}\sum_{\alpha'\in I_{\xi_{\ell-1}}}\sum_{\beta\in I_{\Delta\xi_\ell}}\left(\frac{\partial}{\partial(\boldsymbol{v}_{\ell-1})_{r,\alpha}}(\boldsymbol{\mu}_\ell)_{(\alpha',\beta),i}\right)(\boldsymbol{z}_\ell)_{(\alpha',\beta),i} \tag{273}$$

$$+ \sum_{i=1}^{n_o}\sum_{\alpha'\in I_{\xi_{\ell-2}}}\sum_{\beta\in I_{\Delta\xi_{\ell-1}}}\left(\frac{\partial}{\partial(\boldsymbol{v}_{\ell-1})_{r,\alpha}}(\boldsymbol{\mu}_{\ell-1})_{(\alpha',\beta),i}\right)(\boldsymbol{z}_{\ell-1})_{(\alpha',\beta),i} \tag{274}$$

$$= \sum_{i=1}^{n_o}\sum_{\alpha'\in I_{\xi_{\ell-1}}}\sum_{\beta\in I_{\Delta\xi_\ell}}\left(\frac{\partial}{\partial(\boldsymbol{v}_{\ell-1})_{r,\alpha}}(\boldsymbol{\mu}_\ell)_{(\alpha',\beta),i}\right)(\boldsymbol{z}_\ell)_{(\alpha',\beta),i} \tag{275}$$

$$= \sqrt{\frac{\nu_\ell}{n_o}}\sum_{i=1}^{n_o}\sum_{\beta\in I_{\Delta\xi_\ell}}\left((\boldsymbol{w}_\ell)_{r,1}^{\beta_1}\cdots(\boldsymbol{w}_\ell)_{r,d}^{\beta_d}\left((\boldsymbol{w}_\ell)_r^\top\boldsymbol{y}_i\right)_+^{p-\Delta\xi_\ell}\right)(\boldsymbol{z}_\ell)_{(\alpha,\beta),i}. \tag{276}$$

Then the coefficient of $\left((\boldsymbol{w}_\ell)_r^\top\boldsymbol{y}_i\right)^{p-\Delta\xi_\ell}$ is a homogeneous polynomial

$$\sqrt{\frac{\nu_\ell}{n_o}}\sum_{\beta\in I_{\Delta\xi_\ell}}(\boldsymbol{z}_\ell)_{(\alpha,\beta),i}(\boldsymbol{w}_\ell)_{r,d}^{\beta_d}. \tag{277}$$

Lemma C.2 shows $(\boldsymbol{z}_\ell)_{(\alpha,\beta),i} = 0$ for all $\alpha, \beta$. The induction concludes $\boldsymbol{z} = 0$. $\qquad\square$

In the sequel, $\lambda_0 = \lambda_{\min}(\widehat{\boldsymbol{G}}_{\boldsymbol{v}}) > 0$ is the minimal eigenvalue of $\widehat{\boldsymbol{G}}_{\boldsymbol{v}}$. Furthermore, we assume the occurrence of the following event: there exists $R > 1$ such that $\|(\boldsymbol{w}_\ell)_r(0)\|_2 < R$ and $\|(\boldsymbol{v}_\ell)_r(0)\|_2 < R$ for all $\ell = 0, 1, \ldots, L$ and $r \in [m]$. This assumption is employed solely for the purpose of simplifying propositions and their respective proofs. In the main theorem and its proof, we use the following arguments with regard to the conditional probability of the aforementioned event.

**Proposition C.4.** *Let $\delta > 0$ and*

$$C_1 = \binom{d+k}{k}\binom{d+|\xi|}{|\xi|}^2\nu_{\max}^2\left(\frac{1}{n_o} + \frac{1}{n_b}\right)\sqrt{1 + a_{\max}^4}\left(p^{2|\xi|} + 1\right), \tag{278}$$

$$N_1 = 7\binom{d+k}{d}\binom{d+|\xi|}{d}^2 n_o^2 + 4\binom{d+|\xi|}{d}n_o n_b + n_b^2, \tag{279}$$

*be constants. If $m$ is large enough so that*

$$m \geq \frac{32N_1 C_1^2 R^{4p}}{\lambda_0^2} \log\left(\frac{2N_1}{\delta}\right), \tag{280}$$

*then with the probability of at least $1 - \delta$ over the initialization, we have*

$$\left\|\widehat{\boldsymbol{G}}_{\boldsymbol{v}}\left(\boldsymbol{w}\left(0\right), \boldsymbol{v}\left(0\right)\right) - \widehat{\boldsymbol{G}}_{\boldsymbol{v}}^{\infty}\right\|_2 < \frac{\lambda_0}{4}. \tag{281}$$

*Proof.* We use Hoeffding's inequality to measure the error between $\widehat{\boldsymbol{G}}_{\boldsymbol{v}}\left(\boldsymbol{w}, \boldsymbol{v}\right)$ and $\widehat{\boldsymbol{G}}_{\boldsymbol{v}}^{\infty}$, as Gram matrix $\widehat{\boldsymbol{G}}_{\boldsymbol{v}}^{\infty}$ is the expectation of the sample mean. In the Hoeffding's inequalities (220), (228), (233), (241), (245), (251), and (254), we have

$$\frac{1}{n_o^2}\binom{d+k}{d}^2\binom{d+|\xi|}{d}^4 a_{\max}^4 p^{4|\xi|} \leq C_1^2, \tag{282}$$

$$\frac{\nu_\ell}{n_o^2}\binom{d+|\xi|}{d}^2\left(p^{|\xi|}+1\right)^2 a_{\max}^2 p^{2|\xi|} \leq C_1^2, \tag{283}$$

$$\frac{\nu}{n_o n_b}(d+1)^2\binom{d+|\xi|}{d}^2 a_{\max}^4 p^{2|\xi|+2} \leq C_1^2, \tag{284}$$

$$\frac{\nu_\ell^2}{n_o^2}\left(p^{2|\xi|}+1\right)^2 \leq C_1^2 \tag{285}$$

$$\frac{\nu_\ell \nu_{\ell+1}}{n_o^2}p^{2|\xi|} \leq C_1^2, \tag{286}$$

$$\frac{\nu_1 \nu}{n_o n_b}(d+1)^2 a_{\max}^2 p^{2|\xi|+2} \leq C_1^2, \tag{287}$$

$$\frac{\nu^2}{n_b^2}(d+1)^4 a_{\max}^4 p^4 \leq C_1^2. \tag{288}$$

Consequently, all inequalities induce that each component of $\left|\widehat{\boldsymbol{G}}_{\boldsymbol{v}} - \widehat{\boldsymbol{G}}_{\boldsymbol{v}}^{\infty}\right|$ is greater than $\varepsilon$, with a probability of at most $2\exp\left(-\frac{m\varepsilon^2}{2C_1^2 R^{4p}}\right)$. Since there exists at most $N_1$ nonzero elements in $\left|\widehat{\boldsymbol{G}}_{\boldsymbol{v}} - \widehat{\boldsymbol{G}}_{\boldsymbol{v}}^{\infty}\right|$ by (145),

$$\left|\widehat{\boldsymbol{G}}_{\boldsymbol{v}}\left(\boldsymbol{w}, \boldsymbol{v}\right) - \widehat{\boldsymbol{G}}_{\boldsymbol{v}}^{\infty}\right| \leq \varepsilon \tag{289}$$

holds componentwise, with the probability of at least

$$\left(1 - 2\exp\left(-\frac{m\varepsilon^2}{2C_1^2 R^{4p}}\right)\right)^{N_1} \geq 1 - 2N_1 \exp\left(-\frac{m\varepsilon^2}{2C_1^2 R^{4p}}\right). \tag{290}$$

Then, we have

$$\left\|\widehat{\boldsymbol{G}}_{\boldsymbol{v}}\left(\boldsymbol{w}, \boldsymbol{v}\right) - \widehat{\boldsymbol{G}}_{\boldsymbol{v}}^{\infty}\right\|_2 \leq \left\|\widehat{\boldsymbol{G}}_{\boldsymbol{v}}\left(\boldsymbol{w}, \boldsymbol{v}\right) - \widehat{\boldsymbol{G}}_{\boldsymbol{v}}^{\infty}\right\|_F \leq \varepsilon\sqrt{N_1}. \tag{291}$$

Set $\varepsilon$ and $m$ to satisfy $\varepsilon = \frac{1}{4\sqrt{N_1}}\lambda_0$ and $2N\exp\left(-\frac{m\varepsilon^2}{2C_1^2 R^{4p}}\right) < \delta$. In other words, if

$$m \geq \frac{2C_1^2 R^{4p}}{\varepsilon^2}\log\left(\frac{2N_1}{\delta}\right) \tag{292}$$

$$= \frac{32N_1 C_1^2 R^{4p}}{\lambda_0^2}\log\left(\frac{2N_1}{\delta}\right), \tag{293}$$

then $\left\|\widehat{\boldsymbol{G}}_{\boldsymbol{v}}\left(\boldsymbol{w}, \boldsymbol{v}\right) - \widehat{\boldsymbol{G}}_{\boldsymbol{v}}^{\infty}\right\|_2 \leq \frac{1}{4}\lambda_0$ with probability of at least 1-$\delta$. $\qquad\square$

By the above proposition, the initial Gram matrix $\widehat{\boldsymbol{G}}_{\boldsymbol{v}}\left(\boldsymbol{w}\left(0\right), \boldsymbol{v}\left(0\right)\right)$ is likely to be strictly positive definite, and its smallest eigenvalue remains greater than $\frac{\lambda_0}{2}$, at the beginning of the flow. The following lemma implies that at the early stage of the flow, for which $\boldsymbol{w}$ and $\boldsymbol{v}$ are not far from the initial values, the positive definiteness is preserved, and its smallest eigenvalue remains distinct from zero.

**Lemma C.5.** *Suppose* $\|\tilde{\boldsymbol{w}}_\ell - \boldsymbol{w}_\ell(0)\|_2 \le R_w < R,$ *for*

$$R_w = \frac{\lambda_0}{8\sqrt{2}} \left( \nu_{\max} \sqrt{1 + a_{\max}^4} \binom{d+k}{d} \binom{d+|\xi|}{d}^2 p^{2|\xi|+1} (2R)^{2p-1} \right)^{-1}. \tag{294}$$

*Then*

$$\left\| \widehat{\boldsymbol{G}}_{\boldsymbol{v}}(\tilde{\boldsymbol{w}}, \tilde{\boldsymbol{v}}) - \widehat{\boldsymbol{G}}_{\boldsymbol{v}}(\boldsymbol{w}(0), \boldsymbol{v}(0)) \right\|_2 \le \frac{1}{4}\lambda_0. \tag{295}$$

*Proof.* **1. Case of $\boldsymbol{Q}_{0,0}$:** By the mean value theorem with (151), we can induce that

$$\left( \boldsymbol{Q}_{0,0}(\tilde{\boldsymbol{w}}, \tilde{\boldsymbol{v}}) - \boldsymbol{Q}_{0,0}(\boldsymbol{w}(0), \boldsymbol{v}(0)) \right)_{i_1, i_2} \tag{296}$$

$$\le \left| \boldsymbol{Q}_{0,0}(\tilde{\boldsymbol{w}}, \tilde{\boldsymbol{v}})_{i_1, i_2} - \boldsymbol{Q}_{0,0}(\boldsymbol{w}(0), \tilde{\boldsymbol{v}})_{i_1, i_2} \right| + \left| \boldsymbol{Q}_{0,0}(\tilde{\boldsymbol{w}}, \tilde{\boldsymbol{v}})_{i_1, i_2} - \boldsymbol{Q}_{0,0}(\tilde{\boldsymbol{w}}, \boldsymbol{v}(0))_{i_1, i_2} \right| \tag{297}$$

$$= \left| \frac{\partial}{\partial \boldsymbol{w}} \boldsymbol{Q}_{0,0}(\boldsymbol{w}^*, \boldsymbol{v}(0))_{i_1, i_2} \cdot (\tilde{\boldsymbol{w}} - \boldsymbol{w}(0)) \right| + \left| \frac{\partial}{\partial \boldsymbol{v}} \boldsymbol{Q}_{0,0}(\boldsymbol{w}(0), \boldsymbol{v}^*)_{i_1, i_2} (\tilde{\boldsymbol{v}} - \boldsymbol{v}(0)) \right| \tag{298}$$

$$\le \sum_{\ell=0}^{L} \sum_{r=1}^{m} \left( \left\| \frac{\partial}{\partial (\boldsymbol{w}_\ell)_r} (\boldsymbol{Q}_{0,0})(\boldsymbol{w}^*, \boldsymbol{v}(0))_{i_1, i_2} \right\|_2 \cdot \|(\tilde{\boldsymbol{w}}_\ell)_r - (\boldsymbol{w}_\ell)_r(0)\|_2 \right) \tag{299}$$

$$\le \frac{1}{mn_o} a_{\max}^2 p^{2|\xi|+1} \sum_{\ell=0}^{L} \sum_{r=1}^{m} |I_{\xi_\ell}| |J_{\Delta\xi_{\ell+1}}|^2 \left( \|(\boldsymbol{w}_\ell^*)_r\|_2^{2p-1} \cdot \|(\tilde{\boldsymbol{w}}_\ell)_r - (\boldsymbol{w}_\ell)_r(0)\|_2 \right), \tag{300}$$

for some $\boldsymbol{w}^* = \left[ (\boldsymbol{w}_0^*)_1^\top \;\; \cdots \;\; (\boldsymbol{w}_0^*)_m^\top \;\; \cdots \;\; (\boldsymbol{w}_L^*)_0^\top \;\; \cdots \;\; (\boldsymbol{w}_L^*)_m^\top \right]^\top$ lies on a line between $\tilde{\boldsymbol{w}}$ and $\boldsymbol{w}(0)$ and for some $\boldsymbol{v}^*$ lies on a line connecting $\tilde{\boldsymbol{v}}$ and $\boldsymbol{v}(0)$. Since $\|(\boldsymbol{w}_\ell^*)_r\|_2 \le \|(\tilde{\boldsymbol{w}}_\ell)_r - (\boldsymbol{w}_\ell)_r(0)\|_2 + \|(\boldsymbol{w}_\ell)_r(0)\|_2 \le R_w + R \le 2R$ for each $\ell \in \{0\} \cup [L]$ and $r \in [m]$,

$$\left( \boldsymbol{Q}_{0,0}(\tilde{\boldsymbol{w}}, \tilde{\boldsymbol{v}}) - \boldsymbol{Q}_{0,0}(\boldsymbol{w}(0), \boldsymbol{v}(0)) \right)_{i_1, i_2}$$

$$\le \frac{a_{\max}^2}{mn_o} p^{2|\xi|+1} \sum_{\ell=0}^{L} \sum_{r=1}^{m} |I_{\xi_\ell}| |J_{\Delta\xi_{\ell+1}}|^2 \left( (2R)^{2p-1} \|(\tilde{\boldsymbol{w}}_\ell)_r - (\boldsymbol{w}_\ell)_r(0)\|_2 \right)$$

$$\le \frac{a_{\max}^2}{n_o} p^{2|\xi|+1} \binom{d+|\xi|}{d}^2 \sum_{\ell=0}^{L} |I_{\xi_\ell}| (2R)^{2p-1} R_w \tag{301}$$

$$\le \frac{a_{\max}^2}{n_o} p^{2|\xi|+1} \binom{d+k}{d} \binom{d+|\xi|}{d}^2 (2R)^{2p-1} R_w$$

Therefore, we can attain that

$$\sum_{i_1=1}^{n_o} \sum_{i_2=1}^{n_o} \left( \boldsymbol{Q}_{0,0}(\tilde{\boldsymbol{w}}, \tilde{\boldsymbol{v}}) - \boldsymbol{Q}_{0,0}(\boldsymbol{w}(0), \boldsymbol{v}(0)) \right)_{i_1, i_2}^2$$

$$\le a_{\max}^4 \binom{d+k}{d}^2 \binom{d+|\xi|}{d}^4 2 p^{2(2|\xi|+1)} (2R)^{2(2p-1)} R_w^2. \tag{302}$$

**2. Case of $\boldsymbol{Q}_{0,\ell}$:** Similar to $\boldsymbol{Q}_{0,0}$, the mean value theorem with (159) and (163) gives

$$\left( \boldsymbol{Q}_{0,\ell}(\tilde{\boldsymbol{w}}, \tilde{\boldsymbol{v}}) - \boldsymbol{Q}_{0,\ell}(\boldsymbol{w}(0), \boldsymbol{v}(0)) \right)_{i_1, ((\alpha,\beta), i_2)} \tag{303}$$

$$\le \frac{\sqrt{\nu_{\max}}}{mn_o} a_{\max} \binom{d+|\xi|}{d} p^{2|\xi|+1} \sum_{r=1}^{m} \left\{ \|(\tilde{\boldsymbol{w}}_{\ell-1})_r\|^{2p-1} \|(\tilde{\boldsymbol{w}}_{\ell-1})_r - (\boldsymbol{w}_{\ell-1})_r(0)\|_2 \right. \tag{304}$$

$$\left. + \|(\tilde{\boldsymbol{w}}_\ell)_r\|^{2p-1} \|(\tilde{\boldsymbol{w}}_\ell)_r - (\boldsymbol{w}_\ell)_r(0)\|_2 \right\} \tag{305}$$

$$\le \frac{\sqrt{\nu_{\max}}}{mn_o} a_{\max} \binom{d+|\xi|}{d} p^{2|\xi|+1} \sum_{r=1}^{m} 2 \left( (2R)^{2p-1} R_w \right) \tag{306}$$

$$= \frac{\sqrt{\nu_{\max}}}{n_o} a_{\max} \binom{d+|\xi|}{d} p^{2|\xi|+1} 2 (2R)^{2p-1} R_w. \tag{307}$$

Consequently, we have

$$\sum_{i_1,i_2=1}^{n_o} \sum_{\ell=1}^{L} \sum_{\alpha \in I_{\xi_{\ell-1}}} \sum_{\beta \in I_{\Delta \xi_\ell}} \left( \boldsymbol{Q}_{0,\ell} \left( \tilde{\boldsymbol{w}}, \tilde{\boldsymbol{v}} \right) - \boldsymbol{Q}_{0,\ell} \left( \boldsymbol{w}\left(0\right), \boldsymbol{v}\left(0\right) \right) \right)^2_{i_1, ((\alpha,\beta),i_2)} \tag{308}$$

$$\leq \frac{\nu_{\max}^2}{n_o^2} a_{\max}^2 \binom{d+|\xi|}{d}^2 p^{2(2|\xi|+1)} \sum_{i_1,i_2=1}^{n_o} \sum_{\ell=1}^{L} \sum_{\alpha \in I_{\xi_{\ell-1}}} \sum_{\beta \in I_{\Delta \xi_\ell}} 4 (2R)^{2(2p-1)} R_w^2 \tag{309}$$

$$\leq \nu_{\max}^2 a_{\max}^2 \binom{d+|\xi|}{d}^2 p^{2(2|\xi|+1)} \sum_{\ell=1}^{L} \left| I_{\xi_{\ell-1}} \right| \left| I_{\Delta \xi_\ell} \right| 4 (2R)^{2(2p-1)} R_w^2 \tag{310}$$

$$\leq \nu_{\max}^2 a_{\max}^2 \binom{d+k}{d} \binom{d+|\xi|}{d}^4 4 (2R)^{2(2p-1)} R_w^2. \tag{311}$$

**3. Case of $\boldsymbol{Q}_{0,L+1}$:** The mean value theorem with (171) induces

$$\left( \left( \boldsymbol{Q}_{0,L+1} \right) \left( \tilde{\boldsymbol{w}}, \tilde{\boldsymbol{v}} \right) - \left( \boldsymbol{Q}_{0,L+1} \right) \left( \boldsymbol{w}\left(0\right), \boldsymbol{v}\left(0\right) \right) \right)_{i,j}$$

$$\leq \frac{\sqrt{\nu_{\max}}}{m \sqrt{n_o n_b}} a_{\max}^2 (d+1) \binom{d+|\xi|}{d} p^{|\xi|+2} \sum_{r=1}^{m} \left( \left\| (\tilde{\boldsymbol{w}}_0)_r \right\|_2^{2p-1} \left\| (\tilde{\boldsymbol{w}}_0)_r - (\boldsymbol{w}_0)_r \left(0\right) \right\|_2 \right) \tag{312}$$

$$\leq \frac{\sqrt{\nu_{\max}}}{\sqrt{n_o n_b}} a_{\max}^2 (d+1) \binom{d+|\xi|}{d} p^{|\xi|+2} (2R)^{2p-1} R_w,$$

and hence

$$\sum_{i=1}^{n_o} \sum_{j=1}^{n_b} \left( \left( \boldsymbol{Q}_{0,L+1} \right) \left( \tilde{\boldsymbol{w}}, \tilde{\boldsymbol{v}} \right) - \left( \boldsymbol{Q}_{0,L+1} \right) \left( \boldsymbol{w}\left(0\right), \boldsymbol{v}\left(0\right) \right) \right)^2_{i,j}$$

$$\leq \nu_{\max} a_{\max}^4 (d+1)^2 \binom{d+|\xi|}{d}^2 p^{2(|\xi|+2)} (2R)^{2(2p-1)} R_w^2 \tag{313}$$

**4. Case of $\boldsymbol{Q}_{\ell_1,\ell_2}$:** From the mean value theorem with (180) and (184), we attain

$$\left( \left( \boldsymbol{Q}_{\ell,\ell} \right) \left( \tilde{\boldsymbol{w}}, \tilde{\boldsymbol{v}} \right) - \left( \boldsymbol{Q}_{\ell,\ell} \right) \left( \boldsymbol{w}\left(0\right), \boldsymbol{v}\left(0\right) \right) \right)_{((\alpha_1,\beta_1),i_1),((\alpha_2,\beta_2),i_2)} \tag{314}$$

$$\leq \frac{\nu_{\max}}{m n_o} p^{2|\xi|+1} \sum_{r=1}^{m} \Big\{ \mathbf{1}_{\{\alpha_1=\alpha_2\}} \left\| (\tilde{\boldsymbol{w}}_{\ell-1})_r \right\|_2^{2p-1} \left\| (\tilde{\boldsymbol{w}}_{\ell-1})_r - (\boldsymbol{w}_{\ell-1})_r \left(0\right) \right\|_2 \tag{315}$$

$$+ \mathbf{1}_{\{\alpha_1+\beta_1=\alpha_2+\beta_2\}} \left\| (\tilde{\boldsymbol{w}}_\ell)_r \right\|_2^{2p-1} \left\| (\tilde{\boldsymbol{w}}_\ell)_r - (\boldsymbol{w}_\ell)_r \left(0\right) \right\|_2 \Big\} \tag{316}$$

$$\leq \frac{\nu_{\max}}{n_o} p^{2|\xi|+1} \Big\{ \mathbf{1}_{\{\alpha_1=\alpha_2\}} (2R)^{2p-1} + \mathbf{1}_{\{\alpha_1+\beta_1=\alpha_2+\beta_2\}} (2R)^{2p-1} \Big\} R_w, \tag{317}$$

and further induces

$$\sum_{i_1,i_2=1}^{n_o} \sum_{\ell=1}^{L} \sum_{\alpha_1,\alpha_2 \in I_{\xi_{\ell-1}}} \sum_{\beta_1,\beta_2 \in I_{\Delta \xi_\ell}} \left( \left( \boldsymbol{Q}_{\ell,\ell} \right) \left( \tilde{\boldsymbol{w}}, \tilde{\boldsymbol{v}} \right) - \left( \boldsymbol{Q}_{\ell,\ell} \right) \left( \boldsymbol{w}\left(0\right), \boldsymbol{v}\left(0\right) \right) \right)^2_{((\alpha_1,\beta_1),i_1),((\alpha_2,\beta_2),i_2)}$$

$$\tag{318}$$

$$\leq \nu_{\max}^2 p^{2(2|\xi|+1)} \sum_{\ell=1}^{L} \sum_{\alpha_1 \in I_{\xi_{\ell-1}}} \sum_{\beta_1 \in I_{\Delta \xi_\ell}} \sum_{\alpha_2 \in I_{\xi_{\ell-1}}} \sum_{\beta_2 \in I_{\Delta \xi_\ell}} \Big\{ \mathbf{1}_{\{\alpha_1=\alpha_2\}} (2R)^{2p-1} \tag{319}$$

$$+ \mathbf{1}_{\{\alpha_1+\beta_1=\alpha_2+\beta_2\}} (2R)^{2(2p-1)} \Big\}^2 R_w^2 \tag{320}$$

$$\leq \nu_{\max}^2 p^{2(2|\xi|+1)} \sum_{\ell=1}^{L} \sum_{\alpha_1 \in I_{\xi_{\ell-1}}} \sum_{\beta_1 \in I_{\Delta \xi_\ell}} \left( (|I_{\Delta \xi_\ell}| - 1)(2R)^{2(2p-1)} + (|I_{\Delta \xi_\ell}| - 1)(2R)^{2p-1} \right) \tag{321}$$

$$+4\left(2R\right)^{2(2p-1)}\Big)R_w^2 \tag{322}$$

$$\leq\nu_{\max}^2 p^{2(2|\xi|+1)}\sum_{\ell=1}^{L}\sum_{\alpha_1\in I_{\xi_{\ell-1}}}\sum_{\beta_1\in I_{\Delta\xi_\ell}}4\left|I_{\Delta\xi_\ell}\right|\left(2R\right)^{2(2p-1)}R_w^2 \tag{323}$$

$$\leq 4\nu_{\max}^2 p^{2(2|\xi|+1)}\sum_{\ell=1}^{L}\left|I_{\xi_{\ell-1}}\right|\left|I_{\Delta\xi_\ell}\right|^2\left(2R\right)^{2p-1}R_w^2 \tag{324}$$

$$\leq 4\nu_{\max}^2 p^{2(2|\xi|+1)}\binom{d+k}{d}\binom{d+|\xi|}{d}^2\left(2R\right)^{2p-1}R_w^2. \tag{325}$$

(192) follows that

$$\left(\left(\boldsymbol{Q}_{\ell,\ell+1}\right)\left(\tilde{\boldsymbol{w}},\tilde{\boldsymbol{v}}\right)-\left(\boldsymbol{Q}_{\ell,\ell+1}\right)\left(\boldsymbol{w}\left(0\right),\boldsymbol{v}\left(0\right)\right)\right)_{((\alpha_1,\beta_1),i_1),((\alpha_2,\beta_2),i_2)} \tag{326}$$

$$\leq\mathbf{1}_{\{\alpha_2=\alpha_1+\beta_1\}}\frac{\nu_{\max}}{mn_o}p^{|\xi|+1}\sum_{r=1}^{m}\left(\|(\tilde{\boldsymbol{w}}_\ell)_r\|_2^{2p-1}\|(\tilde{\boldsymbol{w}}_\ell)_r-(\boldsymbol{w}_\ell)_r\left(0\right)\|_2\right) \tag{327}$$

$$\leq\mathbf{1}_{\{\alpha_2=\alpha_1+\beta_1\}}\frac{\nu_{\max}}{n_o}p^{|\xi|+1}\left(2R\right)^{2p-1}R_w. \tag{328}$$

It deduces that

$$\sum_{i_1,i_2=1}^{n_o}\sum_{\ell=1}^{L-1}\sum_{\substack{\alpha_1\in I_{\xi_{\ell-1}}\\\alpha_2\in I_{\xi_\ell}}}\sum_{\substack{\beta_1\in I_{\Delta\xi_\ell}\\\beta_2\in I_{\Delta\xi_{\ell+1}}}}\left(\left(\boldsymbol{Q}_{\ell,\ell+1}\right)\left(\tilde{\boldsymbol{w}},\tilde{\boldsymbol{v}}\right)-\left(\boldsymbol{Q}_{\ell,\ell+1}\right)\left(\boldsymbol{w}\left(0\right),\boldsymbol{v}\left(0\right)\right)\right)_{((\alpha_1,\beta_1),i_1),((\alpha_2,\beta_2),i_2)}^2 \tag{329}$$

$$\leq\nu_{\max}^2 p^{2|\xi|+2}\sum_{\ell=1}^{L-1}\sum_{\alpha_1\in I_{\xi_{\ell-1}}}\sum_{\beta_1\in I_{\Delta\xi_\ell}}\sum_{\alpha_2\in I_{\xi_\ell}}\sum_{\beta_2\in I_{\Delta\xi_{\ell+1}}}\mathbf{1}_{\{\alpha_2=\alpha_1+\beta_1\}}\left(2R\right)^{2(2p-1)}R_w^2 \tag{330}$$

$$\leq\nu_{\max}^2 p^{2|\xi|+2}\sum_{\ell=1}^{L-1}\sum_{\alpha_1\in I_{\xi_{\ell-1}}}\sum_{\beta_1\in I_{\Delta\xi_\ell}}\left|I_{\Delta\xi_{\ell+1}}\right|\left(2R\right)^{2(2p-1)}R_w^2 \tag{331}$$

$$=\nu_{\max}^2 p^{2|\xi|+2}\sum_{\ell=1}^{L-1}\left|I_{\xi_{\ell-1}}\right|\left|I_{\Delta\xi_\ell}\right|\left|I_{\Delta\xi_{\ell+1}}\right|\left(2R\right)^{2(2p-1)}R_w^2 \tag{332}$$

$$\leq\nu_{\max}^2 p^{2|\xi|+2}\binom{d+k}{d}\binom{d+|\xi|}{d}^2\left(2R\right)^{2(2p-1)}R_w^2. \tag{333}$$

In accordance with the definition of the loss $\mu_\ell$, it is clear that $\boldsymbol{Q}_{\ell_1,\ell_2}$ is identical to the zero matrix if the difference between $\ell_1$ and $\ell_2$ is greater than 1. Consequently, together with (325) and (333) we have

$$\sum_{\ell_1,\ell_2=1}^{L}\sum_{\alpha_1,\alpha_2\in I_{\xi_{\ell-1}}}\sum_{\beta_1,\beta_2\in I_{\Delta\xi_\ell}}\sum_{i_1,i_2=1}^{n_o}\left(\left(\boldsymbol{Q}_{\ell_1,\ell_2}\right)\left(\boldsymbol{w}\left(0\right),\boldsymbol{v}\left(0\right)\right)\right)_{((\alpha_1,\beta_1),i_1),((\alpha_2,\beta_2),i_2)}^2 \tag{334}$$

$$=\sum_{i_1,i_2=1}^{n_o}\sum_{\ell=1}^{L}\sum_{\alpha_1,\alpha_2\in I_{\xi_{\ell-1}}}\sum_{\beta_1,\beta_2\in I_{\Delta\xi_\ell}}\left(\left(\boldsymbol{Q}_{\ell,\ell}\right)\left(\tilde{\boldsymbol{w}},\tilde{\boldsymbol{v}}\right)-\left(\boldsymbol{Q}_{\ell,\ell}\right)\left(\boldsymbol{w}\left(0\right),\boldsymbol{v}\left(0\right)\right)\right)_{((\alpha_1,\beta_1),i_1),((\alpha_2,\beta_2),i_2)}^2 \tag{335}$$

$$+2\sum_{i_1,i_2=1}^{n_o}\sum_{\ell=1}^{L-1}\sum_{\substack{\alpha_1\in I_{\xi_{\ell-1}}\\\alpha_2\in I_{\xi_\ell}}}\sum_{\substack{\beta_1\in I_{\Delta\xi_\ell}\\\beta_2\in I_{\Delta\xi_{\ell+1}}}}\left(\left(\boldsymbol{Q}_{\ell,\ell+1}\right)\left(\tilde{\boldsymbol{w}},\tilde{\boldsymbol{v}}\right)-\left(\boldsymbol{Q}_{\ell,\ell+1}\right)\left(\boldsymbol{w}\left(0\right),\boldsymbol{v}\left(0\right)\right)\right)_{((\alpha_1,\beta_1),i_1),((\alpha_2,\beta_2),i_2)}^2 \tag{336}$$

$$\leq 4\nu_{\max}^2 p^{2(2|\xi|+1)}\binom{d+k}{d}\binom{d+|\xi|}{d}^2\left(2R\right)^{2p-1}R_w^2 \tag{337}$$

$$+ 2\nu_{\max}^2 p^{2|\xi|+2} \binom{d+k}{d} \binom{d+|\xi|}{d}^2 (2R)^{2(2p-1)} R_w^2 \tag{338}$$

$$\leq 6\nu_{\max}^2 \binom{d+k}{d} \binom{d+|\xi|}{d}^2 p^{2(2|\xi|+1)} (2R)^{2p-1} R_w^2. \tag{339}$$

**5. Case of $\boldsymbol{Q}_{\ell,L+1}$:** From (201), we have

$$\left( \left( \boldsymbol{Q}_{1,L+1} \right) (\tilde{\boldsymbol{w}}, \tilde{\boldsymbol{v}}) - \left( \boldsymbol{Q}_{1,L+1} \right) (\boldsymbol{w}(0), \boldsymbol{v}(0)) \right)_{((\alpha,\beta),i),j} \tag{340}$$

$$\leq \frac{\nu_{\max}}{m\sqrt{n_o n_b}} a_{\max} (d+1) p^{|\xi|+2} \sum_{r=1}^m \left( \|(\tilde{\boldsymbol{w}}_0)_r\|_2^{2p-1} \|(\tilde{\boldsymbol{w}}_0)_r - (\boldsymbol{w}_0)_r(0)\|_2 \right) \tag{341}$$

$$\leq \frac{\nu_{\max}}{\sqrt{n_o n_b}} a_{\max} (d+1) p^{|\xi|+2} (2R)^{2p-1} R_w. \tag{342}$$

By the definition of the loss $\mu_\ell$, we have $\boldsymbol{Q}_{\ell,L+1}$ is identical to the zero matrix for every $\ell > 1$. Therefore, we have

$$\sum_{i=1}^{n_o} \sum_{j=1}^{n_b} \sum_{\ell=1}^{L} \sum_{\beta \in I_{\Delta\xi_1}} \left( \left( \boldsymbol{Q}_{\ell,L+1} \right) (\tilde{\boldsymbol{w}}, \tilde{\boldsymbol{v}}) - \left( \boldsymbol{Q}_{1,L+1} \right) (\boldsymbol{w}(0), \boldsymbol{v}(0)) \right)_{((1,\beta),i),j}^2 \tag{343}$$

$$= \sum_{i=1}^{n_o} \sum_{j=1}^{n_b} \sum_{\beta \in I_{\Delta\xi_1}} \left( \left( \boldsymbol{Q}_{1,L+1} \right) (\tilde{\boldsymbol{w}}, \tilde{\boldsymbol{v}}) - \left( \boldsymbol{Q}_{1,L+1} \right) (\boldsymbol{w}(0), \boldsymbol{v}(0)) \right)_{((1,\beta),i),j}^2 \tag{344}$$

$$\leq |I_{\Delta\xi_1}| \nu_{\max}^2 a_{\max}^2 (d+1)^2 p^{2(|\xi|+2)} (2R)^{2(2p-1)} R_w^2 \tag{345}$$

$$\leq \nu_{\max}^2 a_{\max}^2 \binom{d+|\xi|-1}{|\xi|} (d+1)^2 p^{2(|\xi|+2)} (2R)^{2(2p-1)} R_w^2 \tag{346}$$

$$\leq \nu_{\max}^2 a_{\max}^2 (d+1)^2 \binom{d+|\xi|}{d} p^{2(|\xi|+2)} (2R)^{2(2p-1)} R_w^2. \tag{347}$$

**6. Case of $\boldsymbol{Q}_{L+1,L+1}$:**

From (205), we have

$$\left( \left( \boldsymbol{Q}_{L+1,L+1} \right) (\tilde{\boldsymbol{w}}, \tilde{\boldsymbol{v}}) - \left( \boldsymbol{Q}_{L+1,L+1} \right) (\boldsymbol{w}(0), \boldsymbol{v}(0)) \right)_{j_1,j_2} \tag{348}$$

$$\leq \frac{\nu}{m n_b} a_{\max}^2 (d+1)^2 p^3 \sum_{r=1}^m \left( \|(\tilde{\boldsymbol{w}}_0)_r\|_2^{2p-1} \|(\tilde{\boldsymbol{w}}_0)_r - (\boldsymbol{w}_0)_r(0)\|_2 \right) \tag{349}$$

$$\leq \frac{\nu_{\max}}{n_b} a_{\max}^2 (d+1)^2 p^3 (2R)^{2p-1} R_w. \tag{350}$$

It further follows that

$$\sum_{j_1=1}^{n_b} \sum_{j_2=1}^{n_b} \left( \left( \boldsymbol{Q}_{L+1,L+1} \right) (\tilde{\boldsymbol{w}}, \tilde{\boldsymbol{v}}) - \left( \boldsymbol{Q}_{L+1,L+1} \right) (\boldsymbol{w}(0), \boldsymbol{v}(0)) \right)_{j_1,j_2}^2 \\ \leq \nu_{\max}^2 a_{\max}^4 (d+1)^4 p^6 (2R)^{2(2p-1)} R_w^2. \tag{351}$$

Therefore, combining (302), (311), (313), (339), (347), and (351) concludes that

$$\left\| \widehat{\boldsymbol{G}}_{\boldsymbol{v}} (\tilde{\boldsymbol{w}}, \tilde{\boldsymbol{v}}) - \widehat{\boldsymbol{G}}_{\boldsymbol{v}} (\boldsymbol{w}(0), \boldsymbol{v}(0)) \right\|_2^2 \tag{352}$$

$$\leq \left\| \widehat{\boldsymbol{G}}_{\boldsymbol{v}} (\tilde{\boldsymbol{w}}, \tilde{\boldsymbol{v}}) - \widehat{\boldsymbol{G}}_{\boldsymbol{v}} (\boldsymbol{w}(0), \boldsymbol{v}(0)) \right\|_F^2 \tag{353}$$

$$\leq a_{\max}^4 \binom{d+k}{d}^2 \binom{d+|\xi|}{d}^4 p^{2(2|\xi|+1)} (2R)^{2(2p-1)} R_w^2 \tag{354}$$

$$+ 4\nu_{\max}^2 a_{\max}^2 \binom{d+k}{d} \binom{d+|\xi|}{d}^4 p^{2(2|\xi|+1)} (2R)^{2(2p-1)} R_w^2 \tag{355}$$

$$+\nu_{\max}a_{\max}^4(d+1)^2\binom{d+|\xi|}{d}^2 p^{2(|\xi|+2)}(2R)^{2(2p-1)}R_w^2 \tag{356}$$

$$+6\nu_{\max}^2 p^{2(2|\xi|+1)}\binom{d+k}{d}\binom{d+|\xi|}{d}^2(2R)^{2p-1}R_w^2 \tag{357}$$

$$+\nu_{\max}^2 a_{\max}^2(d+1)^2\binom{d+|\xi|}{d}p^{2(|\xi|+2)}(2R)^{2(2p-1)}R_w^2 \tag{358}$$

$$+\nu_{\max}^2 a_{\max}^4(d+1)^4 p^6(2R)^{2(2p-1)}R_w^2 \tag{359}$$

$$\leq 8\nu_{\max}^2\left(1+a_{\max}^4\right)\binom{d+k}{d}^2\binom{d+|\xi|}{d}^4 p^{2(2|\xi|+1)}(2R)^{2(2p-1)}R_w^2 \tag{360}$$

$$=\frac{\lambda_0^2}{16}. \tag{361}$$

$\square$

The preceding lemma indicates that the loss will decrease rapidly in cases where $w$ is not significantly distant from $w(0)$. Indeed, the subsequent lemma and proposition demonstrate that $w(t)$ remains within the designated region for any given value of $t > 0$, provided sufficiently large $m$.

**Proposition C.6.** *Set constants $C_2$ and $C_3$ as*

$$C_2=\sqrt{\nu_{\max}}\sqrt{(1+a_{\max}^2)}\binom{d+k}{d}\binom{d+|\xi|}{d}\left(p^{|\xi|}+1\right), \tag{362}$$

$$C_3=\frac{1}{n_o}\sum_{i=1}^{n_o}f(\boldsymbol{x}_i)^2+\frac{1}{n_b}\sum_{j=1}^{n_b}g(\tilde{\boldsymbol{x}}_j)^2, \tag{363}$$

$$N_2=\binom{d+k}{d}\binom{d+|\xi|}{d}n_o+n_b. \tag{364}$$

*For any $C>0$ and $\delta<2N_2\exp\left(-\frac{C_3^2}{2C_2^2 R^{2p}}\right)$, if $m$ is large enough so that*

$$m\geq\frac{2C^2}{\lambda_0^2}\left(C_2\left(4+\binom{d+k}{d}\binom{d+|\xi|}{d}\right)R^{4p}\log\left(\frac{2N_2}{\delta}\right)+C_3R^{2p}\right), \tag{365}$$

*then, with the probability of at least $1-\delta$ over the initialization, we have*

$$\frac{1}{\sqrt{m}}\left\|\begin{bmatrix}\boldsymbol{s}(\boldsymbol{w}(0),\boldsymbol{v}(0))\\\boldsymbol{\mu}_1(\boldsymbol{w}(0),\boldsymbol{v}(0))\\\vdots\\\boldsymbol{\mu}_L(\boldsymbol{w}(0),\boldsymbol{v}(0))\\\boldsymbol{h}(\boldsymbol{w}(0),\boldsymbol{v}(0))\end{bmatrix}\right\|_2<\frac{\lambda_0}{C}R^{-p}. \tag{366}$$

*Proof.* Since

$$\nu_{\max}a_{\max}^2\binom{d+k}{d}^2\binom{d+|\xi|}{d}^2 p^{2|\xi|}\leq C_2^2, \tag{367}$$

$$\nu_{\max}\binom{d+k}{d}^2\binom{d+|\xi|}{d}^2\left(p^{|\xi|}+1\right)^2\leq C_2^2, \tag{368}$$

$$\nu_{\max}a_{\max}^2(d+1)^2 p^2\leq C_2^2, \tag{369}$$

we have

$$\mathbb{P}\left[\left|\sum_{r=1}^m s_{i,r}(\boldsymbol{w}(0),\boldsymbol{v}(0))\right|>\frac{\varepsilon}{\sqrt{n_o}}\right]\leq 2\exp\left(-\frac{\varepsilon^2}{2C_2^2 R^{2p}}\right), \tag{370}$$

$$\mathbb{P}\left[\left|\sum_{r=1}^m(\boldsymbol{\mu}_\ell)_{r,(\alpha,\beta),i}(\boldsymbol{w}(0),\boldsymbol{v}(0))\right|>\frac{\varepsilon}{\sqrt{n_o}}\right]\leq 2\exp\left(-\frac{\varepsilon^2}{2C_2^2 R^{2p}}\right), \tag{371}$$

$$\mathbb{P}\left[\left|\sum_{r=1}^{m} h_{j,r}\left(\boldsymbol{w}\left(0\right),\boldsymbol{v}\left(0\right)\right)\right| > \frac{\varepsilon}{\sqrt{n_b}}\right] \leq 2\exp\left(-\frac{\varepsilon^2}{2C_2^2 R^{2p}}\right). \tag{372}$$

Hence, from

$$n_o + \sum_{\ell=1}^{L}\left|I_{\xi_{\ell-1}}\right|\cdot\left|I_{\Delta\xi_\ell}\right|\cdot n_o + n_b \leq \binom{d+k}{d}\binom{d+|\xi|}{d}n_o + n_b \leq N_2, \tag{373}$$

with the probability of at least

$$\left(1 - 2\exp\left(-\frac{\varepsilon^2}{2C_2^2 R^{2p}}\right)\right)^{N_2} \geq 1 - 2N\exp\left(-\frac{\varepsilon^2}{2C_2^2 R^{2p}}\right), \tag{374}$$

all the conditions

$$\left|\sum_{r=1}^{m} s_{i,r}\left(\boldsymbol{w}\left(0\right),\boldsymbol{v}\left(0\right)\right)\right| \leq \frac{\varepsilon}{\sqrt{n_o}}, \tag{375}$$

$$\left|\sum_{r=1}^{m}\left(\boldsymbol{\mu}_\ell\right)_{r,(\alpha,\beta),i}\left(\boldsymbol{w}\left(0\right),\boldsymbol{v}\left(0\right)\right)\right| \leq \frac{\varepsilon}{\sqrt{n_o}}, \tag{376}$$

$$\left|\sum_{r=1}^{m} h_{j,r}\left(\boldsymbol{w}\left(0\right),\boldsymbol{v}\left(0\right)\right)\right| \leq \frac{\varepsilon}{\sqrt{n_b}} \tag{377}$$

are satisfied. Then, the square of the initial loss is bounded as

$$\left\|\boldsymbol{s}\left(\boldsymbol{w}\left(0\right),\boldsymbol{v}\left(0\right)\right)\right\|_2^2 + \left\|\boldsymbol{h}\left(\boldsymbol{w}\left(0\right),\boldsymbol{v}\left(0\right)\right)\right\|_2^2 + \sum_{\ell=1}^{L}\left\|\boldsymbol{\mu}_\ell\left(\boldsymbol{w}\left(0\right),\boldsymbol{v}\left(0\right)\right)\right\|_2^2 \tag{378}$$

$$= \sum_{i=1}^{n_o} s_i\left(\boldsymbol{w}\left(0\right),\boldsymbol{v}\left(0\right)\right)^2 + \sum_{j=1}^{n_b} h_j\left(\boldsymbol{w}\left(0\right),\boldsymbol{v}\left(0\right)\right)^2 \tag{379}$$

$$+ \sum_{i=1}^{n_o}\sum_{\ell=1}^{L}\sum_{\alpha\in I_{\xi_{\ell-1}}}\sum_{\beta\in I_{\Delta\xi_\ell}}\left(\boldsymbol{\mu}_\ell\right)_{(\alpha,\beta),i}\left(\boldsymbol{w}\left(0\right),\boldsymbol{v}\left(0\right)\right)^2 \tag{380}$$

$$\leq \frac{1}{n_o}\sum_{i=1}^{n_o}\left(\varepsilon + \left|f\left(\boldsymbol{x}_i\right)\right|\right)^2 + \frac{1}{n_b}\sum_{j=1}^{n_b}\left(\varepsilon + \left|g\left(\tilde{\boldsymbol{x}}_j\right)\right|\right)^2 + \sum_{\ell=1}^{L}\left|I_{\xi_{\ell-1}}\right|\cdot\left|I_{\Delta\xi_\ell}\right|\varepsilon^2 \tag{381}$$

$$\leq \frac{1}{n_o}\sum_{i=1}^{n_o}\left(\varepsilon + \left|f\left(\boldsymbol{x}_i\right)\right|\right)^2 + \frac{1}{n_b}\sum_{j=1}^{n_b}\left(\varepsilon + \left|g\left(\tilde{\boldsymbol{x}}_j\right)\right|\right)^2 + \binom{d+k}{d}\binom{d+|\xi|}{d}\varepsilon^2 \tag{382}$$

$$\leq \left(4 + \binom{d+k}{d}\binom{d+|\xi|}{d}\right)\varepsilon^2 + 2\left(\frac{1}{n_o}\sum_{i=1}^{n_o} f\left(\boldsymbol{x}_i\right)^2 + \frac{1}{n_b}\sum_{j=1}^{n_b} g\left(\tilde{\boldsymbol{x}}_j\right)^2\right) \tag{383}$$

$$= \left(4 + \binom{d+k}{d}\binom{d+|\xi|}{d}\right)\varepsilon^2 + 2C_3. \tag{384}$$

For $\delta > 0$ and $C > 0$, if $\varepsilon$ and $m$ satisfy the following inequalities,

$$\delta \geq 2N_2\exp\left(-\frac{\varepsilon^2}{2C_2^2 R^{2p}}\right), \tag{385}$$

$$\varepsilon > C_3, \tag{386}$$

$$m \geq \frac{C^2}{\lambda_0^2}R^{2p}\left(\left(4 + \binom{d+k}{d}\binom{d+|\xi|}{d}\right)\varepsilon^2 + 2C_3\right), \tag{387}$$

then for enough small $\delta$ such that

$$\delta < 2N_2\exp\left(-\frac{C_3^2}{2C_2^2 R^{2p}}\right), \tag{388}$$

we have that with the probability of at least $1 - \delta$,

$$\frac{1}{m}\left(\|\boldsymbol{s}\left(\boldsymbol{w}\left(0\right),\boldsymbol{v}\left(0\right)\right)\|_2^2 + \|\boldsymbol{h}\left(\boldsymbol{w}\left(0\right),\boldsymbol{v}\left(0\right)\right)\|_2^2 + \sum_{\ell=1}^{L}\|\boldsymbol{\mu}_\ell\left(\boldsymbol{w}\left(0\right),\boldsymbol{v}\left(0\right)\right)\|_2^2\right) \tag{389}$$

$$\leq\frac{1}{m}\left(\left(4 + \binom{d+k}{d}\binom{d+|\xi|}{d}\right)\varepsilon^2 + 2C_3\right) \tag{390}$$

$$\leq\frac{\lambda_0^2}{C^2}R^{2p}. \tag{391}$$

In other words, if

$$m \geq \frac{2C^2}{\lambda_0^2}\left(C_2\left(4 + \binom{d+k}{d}\binom{d+|\xi|}{d}\right)R^{4p}\log\left(\frac{2N_2}{\delta}\right) + C_3R^{2p}\right), \tag{392}$$

with the probability of at least $1 - \delta$, we have

$$\frac{1}{\sqrt{m}}\left\|\begin{bmatrix}\boldsymbol{s}\left(\boldsymbol{w}\left(0\right),\boldsymbol{v}\left(0\right)\right)\\\boldsymbol{\mu}_1\left(\boldsymbol{w}\left(0\right),\boldsymbol{v}\left(0\right)\right)\\\vdots\\\boldsymbol{\mu}_L\left(\boldsymbol{w}\left(0\right),\boldsymbol{v}\left(0\right)\right)\\\boldsymbol{h}\left(\boldsymbol{w}\left(0\right),\boldsymbol{v}\left(0\right)\right)\end{bmatrix}\right\|_2 < \frac{\lambda_0}{2C}R^{-p}. \tag{393}$$

$\square$

**Proposition C.7.** *Suppose the conditions* (281) *holds and $R_w$ are given as in Lemma C.5. For*

$$C_4 = 6\sqrt{1 + a_{\max}^2}\sqrt{\nu_{\max}}\binom{d+k}{d}\binom{d+|\xi|}{d}p^{|\xi|+1}2^p\frac{1}{R_w}, \tag{394}$$

*if $m$ is large enough so that*

$$\frac{1}{\sqrt{m}}\left\|\begin{bmatrix}\boldsymbol{s}\left(\boldsymbol{w}\left(0\right),\boldsymbol{v}\left(0\right)\right)\\\boldsymbol{\mu}_1\left(\boldsymbol{w}\left(0\right),\boldsymbol{v}\left(0\right)\right)\\\vdots\\\boldsymbol{\mu}_L\left(\boldsymbol{w}\left(0\right),\boldsymbol{v}\left(0\right)\right)\\\boldsymbol{h}\left(\boldsymbol{w}\left(0\right),\boldsymbol{v}\left(0\right)\right)\end{bmatrix}\right\|_2 < \frac{\lambda_0}{C_4}R^{-p}, \tag{395}$$

*then*

$$\|(\boldsymbol{w}_\ell)_r\left(t\right) - (\boldsymbol{w}_\ell)_r\left(0\right)\|_2 < R_w, \tag{396}$$
$$\|(\boldsymbol{v}_\ell)_r\left(t\right) - (\boldsymbol{v}_\ell)_r\left(0\right)\|_2 < R_w, \tag{397}$$

*for all $t > 0$, $r \in [m]$ and $\ell = 0, \cdots, L$.*

*Proof.* We begin the proof by defining

$$T = \sup\left\{t \geq 0 : \|(\boldsymbol{w}_\ell)_r\left(t\right) - (\boldsymbol{w}_\ell)_r\left(0\right)\|_2 < R_w \text{ and } \|(\boldsymbol{v}_\ell)_r\left(t\right) - (\boldsymbol{v}_\ell)_r\left(0\right)\|_2 < R_w, \forall\ell = 0, \cdots, L\right\}. \tag{398}$$

Since $(\boldsymbol{w}_\ell)_r$ and $(\boldsymbol{v}_\ell)$ are continuous, the above set in supremum is nonempty. For $t \in (0, T)$,

$$\frac{d}{dt}\left\|\begin{bmatrix}\boldsymbol{s}\left(\boldsymbol{w}\left(t\right),\boldsymbol{v}\left(t\right)\right)\\\boldsymbol{\mu}_1\left(\boldsymbol{w}\left(t\right),\boldsymbol{v}\left(t\right)\right)\\\vdots\\\boldsymbol{\mu}_L\left(\boldsymbol{w}\left(t\right),\boldsymbol{v}\left(t\right)\right)\\\boldsymbol{h}\left(\boldsymbol{w}\left(t\right),\boldsymbol{v}\left(t\right)\right)\end{bmatrix}\right\|_2^2 \tag{399}$$

$$= -2\begin{bmatrix}\boldsymbol{s}\left(\boldsymbol{w}\left(t\right),\boldsymbol{v}\left(t\right)\right)\\\boldsymbol{\mu}_1\left(\boldsymbol{w}\left(t\right),\boldsymbol{v}\left(t\right)\right)\\\vdots\\\boldsymbol{\mu}_L\left(\boldsymbol{w}\left(t\right),\boldsymbol{v}\left(t\right)\right)\\\boldsymbol{h}\left(\boldsymbol{w}\left(t\right),\boldsymbol{v}\left(t\right)\right)\end{bmatrix}^\top\left(\widehat{\boldsymbol{G}}_w + \widehat{\boldsymbol{G}}_v\right)\begin{bmatrix}\boldsymbol{s}\left(\boldsymbol{w}\left(t\right),\boldsymbol{v}\left(t\right)\right)\\\boldsymbol{\mu}_1\left(\boldsymbol{w}\left(t\right),\boldsymbol{v}\left(t\right)\right)\\\vdots\\\boldsymbol{\mu}_L\left(\boldsymbol{w}\left(t\right),\boldsymbol{v}\left(t\right)\right)\\\boldsymbol{h}\left(\boldsymbol{w}\left(t\right),\boldsymbol{v}\left(t\right)\right)\end{bmatrix} \tag{400}$$

$$\leq -2 \cdot \frac{1}{2}\lambda_0 \left\| \begin{bmatrix} s\left(w\left(t\right),v\left(t\right)\right) \\ \mu_1\left(w\left(t\right),v\left(t\right)\right) \\ \vdots \\ \mu_L\left(w\left(t\right),v\left(t\right)\right) \\ h\left(w\left(t\right),v\left(t\right)\right) \end{bmatrix} \right\|_2^2, \tag{401}$$

where the last inequality comes from Lemma C.5 and the positive semi-definiteness of $\widehat{G}_w$. This implies that

$$\left\| \begin{bmatrix} s\left(w\left(t\right),v\left(t\right)\right) \\ \mu_1\left(w\left(t\right),v\left(t\right)\right) \\ \vdots \\ \mu_L\left(w\left(t\right),v\left(t\right)\right) \\ h\left(w\left(t\right),v\left(t\right)\right) \end{bmatrix} \right\|_2 \leq \exp\left(-\frac{\lambda_0}{2}t\right) \left\| \begin{bmatrix} s\left(w\left(0\right),v\left(0\right)\right) \\ \mu_1\left(w\left(0\right),v\left(0\right)\right) \\ \vdots \\ \mu_L\left(w\left(0\right),v\left(0\right)\right) \\ h\left(w\left(0\right),v\left(0\right)\right) \end{bmatrix} \right\|_2. \tag{402}$$

For $t \in (0, T)$ and $\ell \in \{0, \cdots, L\}$, from $\left\|(w_\ell)_r(0)\right\|_2 \leq R$ and $\left\|(v_\ell)_r(0)\right\|_2 \leq |I_{\xi_\ell}|$ and (398), we have

$$\left\|(w_\ell)_r(t)\right\|_2 \leq \left\|(w_\ell)_r(t) - (w_\ell)_r(0)\right\|_2 + \left\|(w_\ell)_r(0)\right\|_2 \tag{403}$$
$$< R_w + R \tag{404}$$
$$\leq 2R, \tag{405}$$
$$\left\|(v_\ell)_r(t)\right\|_2 \leq \left\|(v_\ell)_r(t) - (v_\ell)_r(0)\right\|_2 + \left\|(v_\ell)_r(0)\right\|_2 \tag{406}$$
$$< R_w + |I_{\xi_\ell}| \tag{407}$$
$$\leq R. \tag{408}$$

From (44), (57), (52), and (48), we can attain

$$\left\| \frac{\partial s_i\left(w\left(t\right),v\left(t\right)\right)}{\partial\left(w_\ell\right)_r} \right\|_2 \leq \sqrt{\frac{1}{mn_o}}a_{\max}\sqrt{|I_{\xi_\ell}|}\,|J_{\Delta\xi_{\ell+1}}|\,p^{\Delta\xi_{\ell+1}+1}2^{p-1}R^p, \tag{409}$$

$$\left\| \frac{\partial h_j\left(w\left(t\right),v\left(t\right)\right)}{\partial\left(w_0\right)_r} \right\|_2 \leq \sqrt{\frac{\nu_{\max}}{mn_b}}a_{\max}\left(d+1\right)p^2 2^{p-1}R^p, \tag{410}$$

$$\left\| \frac{\partial\left(\mu_\ell\right)_{(\alpha,\beta),i}\left(w,v\right)}{\partial\left(w_\ell\right)_r} \right\|_2 \leq \sqrt{\frac{\nu_{\max}}{mn_o}}p 2^{p-1}R^p, \tag{411}$$

$$\left\| \frac{\partial\left(\mu_\ell\right)_{(\alpha,\beta),i}\left(w,v\right)}{\partial\left(w_{\ell-1}\right)_r} \right\|_2 \leq \sqrt{\frac{\nu_{\max}}{mn_o}}p^{|\xi|+1}2^{p-1}R^p, \tag{412}$$

for all $t \in (0, T)$.

Therefore, we can induce that

$$\left\| \frac{d}{dt}\left(w_\ell\right)_r(t) \right\|_2 \tag{413}$$

$$= \left\| \sum_{i=1}^{n_o} s_i\left(w\left(t\right),v\left(t\right)\right)\cdot\frac{\partial s_i\left(w\left(t\right),v\left(t\right)\right)}{\partial\left(w_\ell\right)_r} + \sum_{j=1}^{n_b} h_j\left(w\left(t\right),v\left(t\right)\right)\cdot\frac{\partial h_j\left(w\left(t\right),v\left(t\right)\right)}{\partial\left(w_\ell\right)_r} \right. \tag{414}$$

$$\left. + \sum_{i=1}^{n_o}\sum_{\ell'=1}^{L}\sum_{\alpha\in I_{\xi_{\ell'-1}}}\sum_{\beta\in I_{\Delta\xi_{\ell'}}}\frac{\partial\left(\mu_{\ell'}\right)_{(\alpha,\beta),i}\left(w\left(t\right),v\left(t\right)\right)}{\partial\left(w_\ell\right)_r}\cdot\left(\mu_{\ell'}\right)_{(\alpha,\beta),i}\left(w\left(t\right),v\left(t\right)\right) \right\|_2 \tag{415}$$

$$\leq \sum_{i=1}^{n_o}\left\| \frac{\partial s_i\left(w\left(t\right),v\left(t\right)\right)}{\partial\left(w_\ell\right)_r} \right\|_2\cdot\left|s_i\left(w\left(t\right),v\left(t\right)\right)\right| \tag{416}$$

$$+ \mathbf{1}_{\{\ell=0\}}\cdot\sum_{j=1}^{n_b}\left\| \frac{\partial h_j\left(w\left(t\right),v\left(t\right)\right)}{\partial\left(w_\ell\right)_r} \right\|_2\cdot\left|h_j\left(w\left(t\right),v\left(t\right)\right)\right| \tag{417}$$

$$+ \left( \mathbf{1}_{\{\ell \neq 0\}} \cdot \sum_{i=1}^{n_o} \sum_{\alpha \in I_{\xi_{\ell-1}}} \sum_{\beta \in I_{\Delta \xi_\ell}} \left\| \frac{\partial (\boldsymbol{\mu}_\ell)_{(\alpha,\beta),i} (\boldsymbol{w}(t), \boldsymbol{v}(t))}{\partial (\boldsymbol{w}_\ell)_r} \right\|_2 \cdot \left| (\boldsymbol{\mu}_\ell)_{(\alpha,\beta),i} (\boldsymbol{w}(t), \boldsymbol{v}(t)) \right| \right.$$
(418)

$$\left. + \mathbf{1}_{\{\ell \neq L\}} \cdot \sum_{i=1}^{n_o} \sum_{\alpha \in I_{\xi_\ell}} \sum_{\beta \in I_{\Delta \xi_{\ell+1}}} \left\| \frac{\partial (\boldsymbol{\mu}_{\ell+1})_{(\alpha,\beta),i} (\boldsymbol{w}(t), \boldsymbol{v}(t))}{\partial (\boldsymbol{w}_\ell)_r} \right\|_2 \cdot \left| (\boldsymbol{\mu}_{\ell+1})_{(\alpha,\beta),i} (\boldsymbol{w}(t), \boldsymbol{v}(t)) \right| \right) .$$
(419)

The first term and second term are bounded by

$$\sum_{i=1}^{n_o} \left\| \frac{\partial s_i (\boldsymbol{w}(t), \boldsymbol{v}(t))}{\partial (\boldsymbol{w}_\ell)_r} \right\|_2 \cdot \left| s_i (\boldsymbol{w}(t), \boldsymbol{v}(t)) \right|$$
(420)

$$\leq \sum_{i=1}^{n_o} \sqrt{\frac{\nu_{\max}}{m n_o}} a_{\max} \sqrt{|I_{\xi_\ell}|} |J_{\Delta \xi_{\ell+1}}| p^{|\xi|+1} 2^{p-1} R^p \cdot \left| s_i (\boldsymbol{w}(t), \boldsymbol{v}(t)) \right|$$
(421)

$$= \sqrt{\frac{\nu_{\max}}{m}} a_{\max} \sqrt{|I_{\xi_\ell}|} |J_{\Delta \xi_{\ell+1}}| p^{|\xi|+1} 2^{p-1} R^p \cdot \frac{1}{\sqrt{n_o}} \sum_{i=1}^{n_o} \left| s_i (\boldsymbol{w}(t), \boldsymbol{v}(t)) \right|$$
(422)

$$\leq \sqrt{\frac{\nu_{\max}}{m}} a_{\max} \sqrt{|I_{\xi_\ell}|} |J_{\Delta \xi_{\ell+1}}| p^{|\xi|+1} 2^{p-1} R^p \cdot \left\| \boldsymbol{s} (\boldsymbol{w}(t), \boldsymbol{v}(t)) \right\|_2 ,$$
(423)

and

$$\sum_{j=1}^{n_b} \left\| \frac{\partial h_j (\boldsymbol{w}(t), \boldsymbol{v}(t))}{\partial (\boldsymbol{w}_\ell)_r} \right\|_2 \cdot \left| h_j (\boldsymbol{w}(t), \boldsymbol{v}(t)) \right|$$
(424)

$$\leq \sqrt{\frac{\nu_{\max}}{m}} a_{\max} (d+1) p^2 2^{p-1} R^p \cdot \frac{1}{\sqrt{n_b}} \sum_{j=1}^{n_b} \left| h_j (\boldsymbol{w}(t), \boldsymbol{v}(t)) \right|$$
(425)

$$\leq \sqrt{\frac{\nu_{\max}}{m}} a_{\max} (d+1) p^2 2^{p-1} R^p \left\| \boldsymbol{h} (\boldsymbol{w}(t), \boldsymbol{v}(t)) \right\|_2 .$$
(426)

The third term is bounded by

$$\sum_{i=1}^{n_o} \sum_{\alpha \in I_{\xi_{\ell-1}}} \sum_{\beta \in I_{\Delta \xi_\ell}} \left\| \frac{\partial (\boldsymbol{\mu}_\ell)_{(\alpha,\beta),i} (\boldsymbol{w}(t), \boldsymbol{v}(t))}{\partial (\boldsymbol{w}_\ell)_r} \right\|_2 \cdot \left| (\boldsymbol{\mu}_\ell)_{(\alpha,\beta),i} (\boldsymbol{w}(t), \boldsymbol{v}(t)) \right|$$
(427)

$$\leq \sqrt{\frac{\nu_{\max}}{m n_o}} p 2^{p-1} R^p \cdot \sum_{i=1}^{n_o} \sum_{\alpha \in I_{\xi_{\ell-1}}} \sum_{\beta \in I_{\Delta \xi_\ell}} \left| (\boldsymbol{\mu}_\ell)_{(\alpha,\beta),i} (\boldsymbol{w}(t), \boldsymbol{v}(t)) \right|$$
(428)

$$\leq \sqrt{\frac{\nu_{\max}}{m}} p 2^{p-1} R^p \sqrt{|I_{\xi_{\ell-1}}| \cdot |I_{\Delta \xi_\ell}|} \cdot \left\| \boldsymbol{\mu}_\ell \right\|_2 ,$$
(429)

and the fourth term is

$$\sum_{i=1}^{n_o} \sum_{\alpha \in I_{\xi_\ell}} \sum_{\beta \in I_{\Delta \xi_{\ell+1}}} \left\| \frac{\partial (\boldsymbol{\mu}_{\ell+1})_{(\alpha,\beta),i} (\boldsymbol{w}(t), \boldsymbol{v}(t))}{\partial (\boldsymbol{w}_\ell)_r} \right\|_2 \cdot \left| (\boldsymbol{\mu}_{\ell+1})_{(\alpha,\beta),i} (\boldsymbol{w}(t), \boldsymbol{v}(t)) \right|$$
(430)

$$\leq \sqrt{\frac{\nu_{\max}}{m n_o}} p^{|\xi|+1} 2^{p-1} R^p \cdot \sum_{i=1}^{n_o} \sum_{\alpha \in I_{\xi_\ell}} \sum_{\beta \in I_{\Delta \xi_{\ell+1}}} \left| (\boldsymbol{\mu}_{\ell+1})_{(\alpha,\beta),i} (\boldsymbol{w}(t), \boldsymbol{v}(t)) \right|$$
(431)

$$\leq \sqrt{\frac{\nu_{\max}}{m}} p^{|\xi|+1} 2^{p-1} R^p \sqrt{|I_{\xi_\ell}| \cdot |I_{\Delta \xi_{\ell+1}}|} \cdot \left\| \boldsymbol{\mu}_{\ell+1} \right\|_2 .$$
(432)

Therefore, for $t \in (0, T)$ and $\ell = \{0, \ldots, L\}$,

$$\left\| \frac{d}{dt} (\boldsymbol{w}_\ell)_r (t) \right\|_2$$
(433)

$$\leq \sqrt{\frac{\nu_{\max}}{m}} a_{\max} \sqrt{\binom{d+k}{d}\binom{d+|\xi|}{d}} p^{|\xi|+1} 2^{p-1} R^p \left\| s\left(w\left(t\right), v\left(t\right)\right)\right\|_2 \tag{434}$$

$$+ \mathbf{1}_{\{\ell=0\}} \cdot \frac{\sqrt{\nu_{\max}}}{\sqrt{m}} a_{\max} \left(d+1\right) p^2 2^{p-1} R^p \left\| h\left(w\left(t\right), v\left(t\right)\right)\right\|_2 \tag{435}$$

$$+ \mathbf{1}_{\{\ell\neq 0\}} \cdot \sqrt{\frac{\nu_{\max}}{m}} \sqrt{\binom{d+k}{d}\binom{d+|\xi|}{d}} p 2^{p-1} R^p \left\| \boldsymbol{\mu}_\ell\left(w\left(t\right), v\left(t\right)\right)\right\|_2 \tag{436}$$

$$+ \mathbf{1}_{\{\ell\neq L\}} \cdot \sqrt{\frac{\nu_{\max}}{m}} \sqrt{\binom{d+k}{d}\binom{d+|\xi|}{d}} p^{|\xi|+1} 2^{p-1} R^p \left\| \boldsymbol{\mu}_{\ell+1}\left(w\left(t\right), v\left(t\right)\right)\right\|_2 \tag{437}$$

$$\leq 3\sqrt{\frac{\nu_{\max}}{m}} \sqrt{1+a_{\max}^2} \sqrt{\binom{d+k}{d}\binom{d+|\xi|}{d}} p^{|\xi|+1} 2^{p-1} R^p \left\| \begin{bmatrix} s\left(w\left(t\right), v\left(t\right)\right) \\ \boldsymbol{\mu}_1\left(w\left(t\right), v\left(t\right)\right) \\ \vdots \\ \boldsymbol{\mu}_L\left(w\left(t\right), v\left(t\right)\right) \\ h\left(w\left(t\right), v\left(t\right)\right) \end{bmatrix} \right\|_2 \tag{438}$$

$$\leq 3\sqrt{1+a_{\max}^2} \sqrt{\frac{\nu_{\max}}{m}} \sqrt{\binom{d+k}{d}\binom{d+|\xi|}{d}} p^{|\xi|+1} 2^{p-1} R^p \exp\left(-\frac{\lambda_0}{2}t\right) \left\| \begin{bmatrix} s\left(w\left(0\right), v\left(0\right)\right) \\ \boldsymbol{\mu}_1\left(w\left(0\right), v\left(0\right)\right) \\ \vdots \\ \boldsymbol{\mu}_L\left(w\left(0\right), v\left(0\right)\right) \\ h\left(w\left(0\right), v\left(0\right)\right) \end{bmatrix} \right\|_2. \tag{439}$$

On the other hand,

$$\left\| \int_0^T \frac{d}{ds}\left(w_\ell\right)_r(s)\, ds \right\|_2^2 = \sum_{q=1}^d \left( \int_0^T \frac{d}{ds}\left(w_\ell\right)_{r,q}(s)\, ds \right)^2 \tag{440}$$

$$= \int_0^T \sum_{q=1}^d \left( \int_0^T \frac{d}{ds}\left(w_\ell\right)_{r,q}(s)\, ds \right) \cdot \frac{d}{ds}\left(w_\ell\right)_{r,q}(s)\, ds \tag{441}$$

$$\leq \int_0^T \left\| \int_0^T \frac{d}{ds}\left(w_\ell\right)_r(s)\, ds \right\|_2 \cdot \left\| \frac{d}{ds}\left(w_\ell\right)(s) \right\|_2 ds \tag{442}$$

$$= \left\| \int_0^T \frac{d}{ds}\left(w_\ell\right)_r(s)\, ds \right\|_2 \cdot \int_0^T \left\| \frac{d}{ds}\left(w_\ell\right)(s) \right\|_2 ds, \tag{443}$$

and hence

$$\left\| \left(w_\ell\right)_r(T) - \left(w_\ell\right)_r(0) \right\|_2 \tag{444}$$

$$= \left\| \int_0^T \frac{d}{ds}\left(w_\ell\right)_r(s)\, ds \right\|_2 \tag{445}$$

$$\leq \int_0^T \left\| \frac{d}{ds}\left(w_\ell\right)_r(s) \right\|_2 ds \tag{446}$$

$$\leq 3\sqrt{1+a_{\max}^2} \sqrt{\frac{\nu_{\max}}{m}} \sqrt{\binom{d+k}{d}\binom{d+|\xi|}{d}} p^{|\xi|+1} 2^{p-1} R^p \int_0^T \exp\left(-\frac{\lambda_0}{2}s\right) ds \left\| \begin{bmatrix} s\left(w\left(0\right), v\left(0\right)\right) \\ \boldsymbol{\mu}_1\left(w\left(0\right), v\left(0\right)\right) \\ \vdots \\ \boldsymbol{\mu}_L\left(w\left(0\right), v\left(0\right)\right) \\ h\left(w\left(0\right), v\left(0\right)\right) \end{bmatrix} \right\|_2 \tag{447}$$

$$\leq \frac{6}{\lambda_0}\sqrt{1+a_{\max}^2}\sqrt{\frac{\nu_{\max}}{m}}\sqrt{\binom{d+k}{d}}\binom{d+|\xi|}{d}p^{|\xi|+1}2^{p-1}R^p\left\|\begin{bmatrix} \boldsymbol{s}\left(\boldsymbol{w}\left(0\right),\boldsymbol{v}\left(0\right)\right) \\ \boldsymbol{\mu}_1\left(\boldsymbol{w}\left(0\right),\boldsymbol{v}\left(0\right)\right) \\ \vdots \\ \boldsymbol{\mu}_L\left(\boldsymbol{w}\left(0\right),\boldsymbol{v}\left(0\right)\right) \\ \boldsymbol{h}\left(\boldsymbol{w}\left(0\right),\boldsymbol{v}\left(0\right)\right) \end{bmatrix}\right\|_2 \tag{448}$$

If $m$ is large enough to satisfy (395), we have

$$\left\|\left(\boldsymbol{w}_\ell\right)_r\left(T\right)-\left(\boldsymbol{w}_\ell\right)_r\left(0\right)\right\|_2 \leq \frac{6\sqrt{1+a_{\max}^2}\sqrt{\nu_{\max}}\sqrt{\binom{d+k}{d}}\binom{d+|\xi|}{d}p^{|\xi|+1}2^{p-1}}{\lambda_0}\cdot\frac{\lambda_0}{C_4}<\frac{1}{2}R_w<R_w. \tag{449}$$

Similarly, we obtain

$$\left\|\frac{d}{dt}\left(\boldsymbol{v}_\ell\right)_r\left(t\right)\right\|_2 \tag{450}$$

$$=\left\|\sum_{i=1}^{n_o}s_i\left(\boldsymbol{w}\left(t\right),\boldsymbol{v}\left(t\right)\right)\cdot\frac{\partial s_i\left(\boldsymbol{w}\left(t\right),\boldsymbol{v}\left(t\right)\right)}{\partial\left(\boldsymbol{v}_\ell\right)_r}+\sum_{j=1}^{n_b}h_j\left(\boldsymbol{w}\left(t\right),\boldsymbol{v}\left(t\right)\right)\cdot\frac{\partial h_j\left(\boldsymbol{w}\left(t\right),\boldsymbol{v}\left(t\right)\right)}{\partial\left(\boldsymbol{v}_\ell\right)_r}\right. \tag{451}$$

$$\left.+\sum_{i=1}^{n_o}\sum_{\ell'=1}^{L}\sum_{\alpha\in I_{\xi_{\ell'}-1}}\sum_{\beta\in I_{\Delta\xi'_\ell}}\frac{\partial\left(\boldsymbol{\mu}_{\ell'}\right)_{(\alpha,\beta),i}\left(\boldsymbol{w}\left(t\right),\boldsymbol{v}\left(t\right)\right)}{\partial\left(\boldsymbol{v}_\ell\right)_r}\cdot\left(\boldsymbol{\mu}_{\ell'}\right)_{(\alpha,\beta),i}\left(\boldsymbol{w}\left(t\right),\boldsymbol{v}\left(t\right)\right)\right\|_2 \tag{452}$$

$$\leq\sum_{i=1}^{n_o}\left|s_i\left(\boldsymbol{w}\left(t\right),\boldsymbol{v}\left(t\right)\right)\right|\cdot\left\|\frac{\partial s_i\left(\boldsymbol{w}\left(t\right),\boldsymbol{v}\left(t\right)\right)}{\partial\left(\boldsymbol{v}_\ell\right)_r}\right\|_2 \tag{453}$$

$$+\mathbf{1}_{\{\ell=0\}}\cdot\sum_{j=1}^{n_b}\left|h_j\left(\boldsymbol{w}\left(t\right),\boldsymbol{v}\left(t\right)\right)\right|\cdot\left\|\frac{\partial h_j\left(\boldsymbol{w}\left(t\right),\boldsymbol{v}\left(t\right)\right)}{\partial\left(\boldsymbol{v}_\ell\right)_r}\right\|_2 \tag{454}$$

$$+\mathbf{1}_{\{\ell\neq0\}}\cdot\sum_{i=1}^{n_o}\sum_{\alpha\in I_{\xi_{\ell-1}}}\sum_{\beta\in I_{\Delta\xi_\ell}}\left\|\frac{\partial\left(\boldsymbol{\mu}_\ell\right)_{(\alpha,\beta),i}\left(\boldsymbol{w}\left(t\right),\boldsymbol{v}\left(t\right)\right)}{\partial\left(\boldsymbol{v}_\ell\right)_r}\right\|_2\cdot\left|\left(\boldsymbol{\mu}_\ell\right)_{(\alpha,\beta),i}\left(\boldsymbol{w}\left(t\right),\boldsymbol{v}\left(t\right)\right)\right| \tag{455}$$

$$+\mathbf{1}_{\{\ell\neq L\}}\cdot\sum_{i=1}^{n_o}\sum_{\alpha\in I_{\xi_\ell}}\sum_{\beta\in I_{\Delta\xi_{\ell+1}}}\left\|\frac{\partial\left(\boldsymbol{\mu}_{\ell+1}\right)_{(\alpha,\beta),i}\left(\boldsymbol{w}\left(t\right),\boldsymbol{v}\left(t\right)\right)}{\partial\left(\boldsymbol{v}_\ell\right)_r}\right\|_2\cdot\left|\left(\boldsymbol{\mu}_{\ell+1}\right)_{(\alpha,\beta),i}\left(\boldsymbol{w}\left(t\right),\boldsymbol{v}\left(t\right)\right)\right|. \tag{456}$$

The first term and second term are bounded by

$$\sum_{i=1}^{n_o}\left|s_i\left(\boldsymbol{w}\left(t\right),\boldsymbol{v}\left(t\right)\right)\right|\cdot\left\|\frac{\partial s_i\left(\boldsymbol{w}\left(t\right),\boldsymbol{v}\left(t\right)\right)}{\partial\left(\boldsymbol{v}_\ell\right)_r}\right\|_2 \tag{457}$$

$$\leq\sqrt{\frac{\nu_{\max}}{m}}a_{\max}\sqrt{\binom{d+k}{d}}\binom{d+|\xi|}{d}p^{|\xi|}\left(2R\right)^p\frac{1}{\sqrt{n_o}}\sum_{i=1}^{n_o}\left|s_i\left(\boldsymbol{w}\left(t\right),\boldsymbol{v}\left(t\right)\right)\right| \tag{458}$$

$$\leq\sqrt{\frac{\nu_{\max}}{m}}a_{\max}\sqrt{\binom{d+k}{d}}\binom{d+|\xi|}{d}p^{|\xi|}\left(2R\right)^p\left\|\hat{\boldsymbol{s}}\right\|_2, \tag{459}$$

and

$$\sum_{j=1}^{n_b}\left|h_j\left(\boldsymbol{w}\left(t\right),\boldsymbol{v}\left(t\right)\right)\right|\cdot\left\|\frac{\partial h_j\left(\boldsymbol{w}\left(t\right),\boldsymbol{v}\left(t\right)\right)}{\partial\left(\boldsymbol{v}_\ell\right)_r}\right\|_2 \tag{460}$$

$$\leq\sqrt{\frac{\nu_{\max}}{m}}\left(d+1\right)a_{\max}p\left(2R\right)^p\frac{1}{\sqrt{n_b}}\sum_{j=1}^{n_b}\left|h_j\left(\boldsymbol{w}\left(t\right),\boldsymbol{v}\left(t\right)\right)\right| \tag{461}$$

$$\leq \sqrt{\frac{\nu_{\max}}{m}} \left(d+1\right) a_{\max} p \left(2R\right)^p \left\|\boldsymbol{h}\right\|_2. \tag{462}$$

The third term is bounded by

$$\sum_{i=1}^{n_o} \sum_{\alpha \in I_{\xi_{\ell-1}}} \sum_{\beta \in I_{\Delta\xi_\ell}} \left\|\frac{\partial \left(\boldsymbol{\mu}_\ell\right)_{(\alpha,\beta),i} \left(\boldsymbol{w}\left(t\right), \boldsymbol{v}\left(t\right)\right)}{\partial \left(\boldsymbol{v}_\ell\right)_r}\right\|_2 \cdot \left|\left(\boldsymbol{\mu}_\ell\right)_{(\alpha,\beta),i} \left(\boldsymbol{w}\left(t\right), \boldsymbol{v}\left(t\right)\right)\right| \tag{463}$$

$$\leq \sqrt{\frac{\nu_{\max}}{m n_o}} \sqrt{\binom{d+k}{d}\binom{d+|\xi|}{d}} \left(2R\right)^p \cdot \sum_{i=1}^{n_o} \sum_{\alpha \in I_{\xi_{\ell-1}}} \sum_{\beta \in I_{\Delta\xi_\ell}} \left|\left(\boldsymbol{\mu}_\ell\right)_{(\alpha,\beta),i} \left(\boldsymbol{w}\left(t\right), \boldsymbol{v}\left(t\right)\right)\right| \tag{464}$$

$$\leq \sqrt{\frac{\nu_{\max}}{m}} \sqrt{\binom{d+k}{d}\binom{d+|\xi|}{d}} \left(2R\right)^p \sqrt{\left|I_{\xi_{\ell-1}}\right| \cdot \left|I_{\Delta\xi_\ell}\right|} \cdot \left\|\boldsymbol{\mu}_\ell\right\|_2 \tag{465}$$

$$\leq \sqrt{\frac{\nu_{\max}}{m}} \binom{d+k}{d}\binom{d+|\xi|}{d} \left(2R\right)^p \left\|\boldsymbol{\mu}_\ell\right\|_2, \tag{466}$$

and the fourth term is

$$\sum_{i=1}^{n_o} \sum_{\alpha \in I_{\xi_\ell}} \sum_{\beta \in I_{\Delta\xi_{\ell+1}}} \left\|\frac{\partial \left(\boldsymbol{\mu}_{\ell+1}\right)_{(\alpha,\beta),i} \left(\boldsymbol{w}\left(t\right), \boldsymbol{v}\left(t\right)\right)}{\partial \left(\boldsymbol{v}_\ell\right)_r}\right\|_2 \cdot \left|\left(\boldsymbol{\mu}_{\ell+1}\right)_{(\alpha,\beta),i} \left(\boldsymbol{w}\left(t\right), \boldsymbol{v}\left(t\right)\right)\right| \tag{467}$$

$$\leq \sqrt{\frac{\nu_{\max}}{m n_o}} p^{|\xi|} \sqrt{\binom{d+k}{d}\binom{d+|\xi|}{d}} \left(2R\right)^p \cdot \sum_{\alpha \in I_{\xi_\ell}} \sum_{\beta \in I_{\Delta\xi_{\ell+1}}} \left|\left(\boldsymbol{\mu}_{\ell+1}\right)_{(\alpha,\beta),i} \left(\boldsymbol{w}\left(t\right), \boldsymbol{v}\left(t\right)\right)\right| \tag{468}$$

$$\leq \sqrt{\frac{\nu_{\max}}{m}} \binom{d+k}{d}\binom{d+|\xi|}{d} p^{|\xi|} \left(2R\right)^p \left\|\boldsymbol{\mu}_{\ell+1}\right\|_2. \tag{469}$$

Therefore, for $t \in (0,T)$ and $\ell = \{0, \ldots, L\}$,

$$\left\|\frac{d}{dt} \left(\boldsymbol{v}_\ell\right)_r (t)\right\|_2 \tag{470}$$

$$\leq \sqrt{\frac{\nu_{\max}}{m}} a_{\max} \sqrt{\binom{d+k}{d}\binom{d+|\xi|}{d}} p^{|\xi|} \left(2R\right)^p \left\|\boldsymbol{s}\right\|_2 \tag{471}$$

$$+ \mathbf{1}_{\{\ell=0\}} \cdot \sqrt{\frac{\nu_{\max}}{m}} \left(d+1\right) a_{\max} p \left(2R\right)^p \left\|\boldsymbol{h}\right\|_2 \tag{472}$$

$$+ \mathbf{1}_{\{\ell\neq 0\}} \cdot \sqrt{\frac{\nu_{\max}}{m}} \binom{d+k}{d}\binom{d+|\xi|}{d} \left(2R\right)^p \left\|\boldsymbol{\mu}_\ell\right\|_2 \tag{473}$$

$$+ \mathbf{1}_{\{\ell\neq L\}} \cdot \sqrt{\frac{\nu_{\max}}{m}} \binom{d+k}{d}\binom{d+|\xi|}{d} p^{|\xi|} \left(2R\right)^p \left\|\boldsymbol{\mu}_{\ell+1}\right\|_2 \tag{474}$$

$$\leq 3 \sqrt{\frac{\nu_{\max}}{m}} \sqrt{1+a_{\max}^2} \binom{d+k}{d}\binom{d+|\xi|}{d} p^{|\xi|} \left(2R\right)^p \left\|\begin{bmatrix} \boldsymbol{s}\left(\boldsymbol{w}\left(t\right), \boldsymbol{v}\left(t\right)\right) \\ \boldsymbol{\mu}_1\left(\boldsymbol{w}\left(t\right), \boldsymbol{v}\left(t\right)\right) \\ \vdots \\ \boldsymbol{\mu}_L\left(\boldsymbol{w}\left(t\right), \boldsymbol{v}\left(t\right)\right) \\ \boldsymbol{h}\left(\boldsymbol{w}\left(t\right), \boldsymbol{v}\left(t\right)\right) \end{bmatrix}\right\|_2 \tag{475}$$

$$\leq 3 \sqrt{\frac{\nu_{\max}}{m}} \sqrt{1+a_{\max}^2} \binom{d+k}{d}\binom{d+|\xi|}{d} p^{|\xi|} \left(2R\right)^p \exp\left(-\frac{\lambda_0}{2}t\right) \left\|\begin{bmatrix} \boldsymbol{s}\left(\boldsymbol{w}\left(0\right), \boldsymbol{v}\left(0\right)\right) \\ \boldsymbol{\mu}_1\left(\boldsymbol{w}\left(0\right), \boldsymbol{v}\left(0\right)\right) \\ \vdots \\ \boldsymbol{\mu}_L\left(\boldsymbol{w}\left(0\right), \boldsymbol{v}\left(0\right)\right) \\ \boldsymbol{h}\left(\boldsymbol{w}\left(0\right), \boldsymbol{v}\left(0\right)\right) \end{bmatrix}\right\|_2 \tag{476}$$

As a consequence, we attain

$$\left\|\left(\boldsymbol{v}_\ell\right)_r (T) - \left(\boldsymbol{v}_\ell\right)_r (0)\right\|_2 \tag{477}$$

$$\leq \int_0^T \left\| \frac{d}{ds} \left( \boldsymbol{v}_\ell \right)_r (s) \right\|_2 ds \tag{478}$$

$$\leq 3 \sqrt{\frac{\nu_{\max}}{m}} \sqrt{1 + a_{\max}^2} \binom{d+k}{d} \binom{d+|\xi|}{d} p^{|\xi|} (2R)^p \int_0^T \exp\left( -\frac{\lambda_0}{2} s \right) ds \left\| \begin{bmatrix} \boldsymbol{s} \left( \boldsymbol{w}(0), \boldsymbol{v}(0) \right) \\ \boldsymbol{\mu}_1 \left( \boldsymbol{w}(0), \boldsymbol{v}(0) \right) \\ \vdots \\ \boldsymbol{\mu}_L \left( \boldsymbol{w}(0), \boldsymbol{v}(0) \right) \\ \boldsymbol{h} \left( \boldsymbol{w}(0), \boldsymbol{v}(0) \right) \end{bmatrix} \right\|_2 \tag{479}$$

$$\leq \frac{6}{\lambda_0} \sqrt{1 + a_{\max}^2} \sqrt{\frac{\nu_{\max}}{m}} \binom{d+k}{d} \binom{d+|\xi|}{d} p^{|\xi|} (2R)^p \left\| \begin{bmatrix} \boldsymbol{s} \left( \boldsymbol{w}(0), \boldsymbol{v}(0) \right) \\ \boldsymbol{\mu}_1 \left( \boldsymbol{w}(0), \boldsymbol{v}(0) \right) \\ \vdots \\ \boldsymbol{\mu}_L \left( \boldsymbol{w}(0), \boldsymbol{v}(0) \right) \\ \boldsymbol{h} \left( \boldsymbol{w}(0), \boldsymbol{v}(0) \right) \end{bmatrix} \right\|_2. \tag{480}$$

Since $m$ is large enough to satisfy (395), we have

$$\left\| \left( \boldsymbol{v}_\ell \right)_r (T) - \left( \boldsymbol{v}_\ell \right)_r (0) \right\|_2 \leq \frac{6 \sqrt{1 + a_{\max}^2} \sqrt{\nu_{\max}} \binom{d+k}{d} \binom{d+|\xi|}{d} p^{|\xi|} 2^p}{\lambda_0} \cdot \frac{\lambda_0}{C_4} = \frac{1}{p} R_w < R_w. \tag{481}$$

$\square$

## C.1   Proof of Theorem 4.3

*Proof.* For $R \in \mathbb{R}$, $r \in [m]$, and $q \in [d]$, each $\left( \boldsymbol{w}_\ell \right)_{r,q}$ satisfies

$$\mathbb{P} \left[ \left| \left( \boldsymbol{w}_\ell \right)_{r,q} \right| > R \right] < \exp\left( -\frac{1}{2} R^2 \right). \tag{482}$$

If we set $\delta_1 = (L+1) md \exp\left( -\frac{1}{2} R^2 \right)$, then $\delta_1$ satisfies

$$\left( 1 - \exp\left( -\frac{1}{2} R^2 \right) \right)^{(L+1)md} \geq 1 - (L+1) md \exp\left( -\frac{1}{2} R^2 \right) = 1 - \delta_1. \tag{483}$$

Hence, with probability of at least $1 - \delta_1$, we have $\left| \left( \boldsymbol{w}_\ell \right)_{r,q} \right| \leq R$ for all $\ell = \{0, 1, \ldots, L+1\}$, $r \in [m]$, and $q \in [d]$.

Provided $\left| \left( \boldsymbol{w}_\ell \right)_{r,q} \right| \leq R$ for all $\ell = \{0, 1, \ldots, L+1\}$, if

$$m \geq \frac{32 N_1 C_1^2 R^{4p}}{\lambda_0^2} \log\left( \frac{2N_1}{\delta_2} \right), \tag{484}$$

where

$$C_1 = \binom{d+k}{k} \binom{d+|\xi|}{|\xi|}^2 \nu_{\max}^2 \left( \frac{1}{n_o} + \frac{1}{n_b} \right) \sqrt{1 + a_{\max}^4} \left( p^{2|\xi|} + 1 \right), \tag{485}$$

$$N_1 = 7 \binom{d+k}{d} \binom{d+|\xi|}{d}^2 n_o^2 + 4 \binom{d+|\xi|}{d} n_o n_b + n_b^2, \tag{486}$$

then by Proposition C.4, with probability of at least $1 - \delta_2$, we have

$$\left\| \widehat{\boldsymbol{G}}_{\boldsymbol{v}} \left( \boldsymbol{w}(0), \boldsymbol{v}(0) \right) - \widehat{\boldsymbol{G}}_{\boldsymbol{v}}^\infty \right\|_2 < \frac{\lambda_0}{4}. \tag{487}$$

On the other hand, with the condition $\left| \left( \boldsymbol{w}_\ell \right)_{r,q} \right| \leq R$ for all $\ell = \{0, 1, \ldots, L+1\}$, $r \in [m]$, and $q \in [d]$, Proposition C.6 shows that if

$$m \geq \frac{2C_4^2}{\lambda_0^2} \left( C_2 \left( 4 + \binom{d+k}{d} \binom{d+|\xi|}{d} \right) R^{4p} \log\left( \frac{2N_2}{\delta} \right) + C_3 R^{2p} \right), \tag{488}$$

$$\delta_3 \le 2N_2 \exp\left(-\frac{C_3^2}{2C_2^2 R^{2p}}\right), \tag{489}$$

$$R_w \le R, \tag{490}$$

where

$$R_w = \frac{\lambda_0}{8\sqrt{2}} \left(\nu_{\max}\sqrt{1 + a_{\max}^4} \binom{d+k}{d} \binom{d+|\xi|}{d}^2 p^{2|\xi|+1} (2R)^{2p-1}\right)^{-1}, \tag{491}$$

$$C_2 = \sqrt{\nu_{\max}} \sqrt{(1 + a_{\max}^2)} \binom{d+k}{d} \binom{d+|\xi|}{d} \left(p^{|\xi|} + 1\right), \tag{492}$$

$$C_3 = \frac{1}{n_o} \sum_{i=1}^{n_o} f(\boldsymbol{x}_i)^2 + \frac{1}{n_b} \sum_{j=1}^{n_b} g(\tilde{\boldsymbol{x}}_j)^2, \tag{493}$$

$$C_4 = \sqrt{\nu_{\max}} \sqrt{1 + a_{\max}^2} \binom{d+k}{d} \binom{d+|\xi|}{d} p^{|\xi|+1} 2^p \frac{1}{R_w}, \tag{494}$$

$$N_2 = \binom{d+k}{d} \binom{d+|\xi|}{d} n_o + n_b, \tag{495}$$

then, with the probability of at least $1 - \delta_3$, we have

$$\frac{1}{\sqrt{m}} \left\| \begin{bmatrix} \boldsymbol{s}(\boldsymbol{w}(0), \boldsymbol{v}(0)) \\ \boldsymbol{\mu}_1(\boldsymbol{w}(0), \boldsymbol{v}(0)) \\ \vdots \\ \boldsymbol{\mu}_L(\boldsymbol{w}(0), \boldsymbol{v}(0)) \\ \boldsymbol{h}(\boldsymbol{w}(0), \boldsymbol{v}(0)) \end{bmatrix} \right\|_2 < \frac{\lambda_0}{C_4} R^{-p}. \tag{496}$$

Consequently, Proposition C.7 implies

$$\|(\boldsymbol{w}_\ell)_r(t) - (\boldsymbol{w}_\ell)_r(0)\|_2 < R_w, \tag{497}$$

$$\|(\boldsymbol{v}_\ell)_r(t) - (\boldsymbol{v}_\ell)_r(0)\|_2 < R_w, \tag{498}$$

for all $t \ge 0$. Then, we have

$$\left\| \begin{bmatrix} \boldsymbol{s}(\boldsymbol{w}(t), \boldsymbol{v}(t)) \\ \boldsymbol{\mu}_1(\boldsymbol{w}(t), \boldsymbol{v}(t)) \\ \vdots \\ \boldsymbol{\mu}_L(\boldsymbol{w}(t), \boldsymbol{v}(t)) \\ \boldsymbol{h}(\boldsymbol{w}(t), \boldsymbol{v}(t)) \end{bmatrix} \right\|_2 \le \exp\left(-\frac{\lambda_0}{2}t\right) \left\| \begin{bmatrix} \boldsymbol{s}(\boldsymbol{w}(0), \boldsymbol{v}(0)) \\ \boldsymbol{\mu}_1(\boldsymbol{w}(0), \boldsymbol{v}(0)) \\ \vdots \\ \boldsymbol{\mu}_L(\boldsymbol{w}(0), \boldsymbol{v}(0)) \\ \boldsymbol{h}(\boldsymbol{w}(0), \boldsymbol{v}(0)) \end{bmatrix} \right\|_2 \tag{499}$$

from Lemam C.5.

Set $\delta_1 = \delta_2 = \delta_3 = \frac{1}{3}\delta$ and consider all inequality conditions. First, for $\delta_1 = \frac{1}{3}\delta$,

$$R = \sqrt{\log \frac{6(L+1)md}{\delta}} \tag{500}$$

satisfies the condition (483). Then, (484) is satisfied if

$$m \ge \frac{32N_1 C_1^2}{\lambda_0^2} \left(\log \frac{6(L+1)md}{\delta}\right)^{2p} \log\left(\frac{6N_1}{\delta}\right). \tag{501}$$

Similarly, if $m$ is large enough so that

$$m \ge \frac{2C_2 C_4^2}{\lambda_0^2} \left(4 + \binom{d+k}{d} \binom{d+|\xi|}{d}\right) \left(\log \frac{6(L+1)md}{\delta}\right)^{2p} \log\left(\frac{2N_2}{\delta}\right) \tag{502}$$

$$+ \frac{2C_3 C_4^2}{\lambda_0^2} \left(\log \frac{6(L+1)md}{\delta}\right)^p, \tag{503}$$

then (488) holds.

Therefore, there exsits $C_5 = C_5\left(a_{\max}, \nu_{\max}, n_o, n_b, \lambda_0\right)$ such that if

$$m \geq C\left(\frac{d+k}{d}\right)^6 \left(\frac{d+|\xi|}{d}\right)^8 p^{7|\xi|+4} 2^{6p} \left(\log \frac{md}{\delta}\right)^{4p} \tag{504}$$

then with the probability of at least $(1-\delta_1)(1-\delta_2)(1-\delta_3) \geq 1 - (\delta_1 + \delta_2 + \delta_3) = 1 - \delta$, we have

$$\left\| \begin{bmatrix} s\left(\boldsymbol{w}(t), \boldsymbol{v}(t)\right) \\ \boldsymbol{\mu}_1\left(\boldsymbol{w}(t), \boldsymbol{v}(t)\right) \\ \vdots \\ \boldsymbol{\mu}_L\left(\boldsymbol{w}(t), \boldsymbol{v}(t)\right) \\ \boldsymbol{h}\left(\boldsymbol{w}(t), \boldsymbol{v}(t)\right) \end{bmatrix} \right\|_2 \leq \exp\left(-\frac{\lambda_0}{2}t\right) \left\| \begin{bmatrix} s\left(\boldsymbol{w}(0), \boldsymbol{v}(0)\right) \\ \boldsymbol{\mu}_1\left(\boldsymbol{w}(0), \boldsymbol{v}(0)\right) \\ \vdots \\ \boldsymbol{\mu}_L\left(\boldsymbol{w}(0), \boldsymbol{v}(0)\right) \\ \boldsymbol{h}\left(\boldsymbol{w}(0), \boldsymbol{v}(0)\right) \end{bmatrix} \right\|_2. \tag{505}$$

$\square$

# D   Experimental Details

## D.1   Problem Formulations

**Poisson equation**

$$\begin{cases} u_{xx} + u_{yy} = f_1 & x, y \in (0, \pi) \\ u(x, 0) = u(x, \pi) = u(0, y) = u(\pi, y) = 0 & x, y \in [0, \pi], \end{cases} \tag{506}$$

where $f_1 : (0, \pi)^2 \to \mathbb{R}$ is defined so that the exact solution is given by

$$u(x, y) = \frac{1}{5} \sin^2(x) y^2 (\pi - y)^2. \tag{507}$$

**Bi-harmonic equation**

$$\begin{cases} u_{xxxx} + 2u_{xxyy} + u_{yyyy} = f_2 & x, y \in (0, \pi) \\ u(x, 0) = u(x, \pi) = u(0, y) = u(\pi, y) = 0 & x, y \in [0, \pi] \\ \frac{\partial}{\partial \mathbf{n}} u(x, 0) = \frac{\partial}{\partial \mathbf{n}} u(x, \pi) = \frac{\partial}{\partial \mathbf{n}} u(0, y) = \frac{\partial}{\partial \mathbf{n}} u(\pi, y) = 0 & x, y \in (0, \pi), \end{cases} \tag{508}$$

where the flux function $f_2 : (0, \pi)^2 \to \mathbb{R}$ is set to ensure that it has a solution that is identical to that of Poisson equation (507).

**Heat equation**

$$\begin{cases} u_t = u_{xx} & x \in (-1, 1), \ t \in (0, 1) \\ u(t, -1) = u(t, 1) = 0 & t \in [0, 1] \\ u(0, x) = \sin(\pi x) & x \in (-1, 1), \end{cases} \tag{509}$$

**Elastic beam equation**

$$\begin{cases} u_t + u_{xxxx} = 0 & x \in (0, \pi), \ t \in (0, 1) \\ u(t, 0) = u(t, \pi) = u_{xx}(t, 0) = u_{xx}(t, \pi) = 0 & t \in [0, 1] \\ u(0, x) = 2\sin(x) & x \in (0, \pi). \end{cases} \tag{510}$$

The exact solution is given by $u(t, x) = 2e^{-t}\sin(x)$.

**Convection-diffusion equation**

$$\begin{cases} u_t + u_x - \frac{1}{4}u_{xx} = 0 & (t, x) \in \Omega = (0, 1) \times (0, \pi), \\ u(0, x) = \sin(x) & x \in [0, \pi], \\ u(t, 0) = -e^{-\frac{1}{4}t}\sin(t) & t \in [0, 1], \\ u(t, \pi) = e^{-\frac{1}{4}t}\sin(\pi - t) & t \in [0, 1], \end{cases} \tag{511}$$

whose exact solution is $u(t, x) = e^{-\frac{1}{4}t}\sin(x - t)$.

Table 2: Experimental settings for each PDE

| PDE | width $m$ | power $p$ | optimizer(lr) | $n_o$ | $n_b$ |
|---|---|---|---|---|---|
| Harmonic | $10^2 \sim 10^6$ | $5 \sim 8$ | GD($10^{-8}$) | 400 | 400 |
| Bi-harmonic | $10^2 \sim 10^6$ | $5 \sim 8$ | GD($10^{-8}$) | 400 | 400 |
| Heat | $10^3$ | $3 \sim 10$ | Adam($10^{-3}$) | 300 | 300 |
| Heat(PINNs) | $10^3$ | 3 | GD($10^{-1}$) | 300 | 300 |
| Heat(VS-PINNs) | $10^3$ | 2 | GD($10^{-1}$) | 300 | 300 |
| Beam (PINNs) | $10^3$ | 5 | Adam($10^{-3}$) | 10,000 | 300 |
| Beam (VS-PINNs $|\xi| = 2$) | $10^3$ | 3 | Adam($10^{-3}$) | 10,000 | 300 |
| Beam (VS-PINNs $|\xi| = 1$) | $10^3$ | 2 | Adam($10^{-3}$) | 10,000 | 300 |
| Convection-diffusion (PINNs) | $10^3$ | 3 | Adam($10^{-3}$) / GD($10^{-2}$) | 300 | 300 |
| Convection-diffusion (VS-PINNs) | $10^3$ | 2 | Adam($10^{-3}$) / GD($10^{-1}$) | 300 | 300 |
| Bi-harmonic (PINNs) | $10^3$ | 5 | Adam($10^{-3}$) | 10,000 | 400 |
| Bi-harmonic (VS-PINNs $|\xi| = 2$) | $10^3$ | 3 | Adam($10^{-3}$) | 10,000 | 400 |
| Bi-harmonic (VS-PINNs $|\xi| = 1$) | $10^3$ | 2 | Adam($10^{-3}$) | 10,000 | 400 |

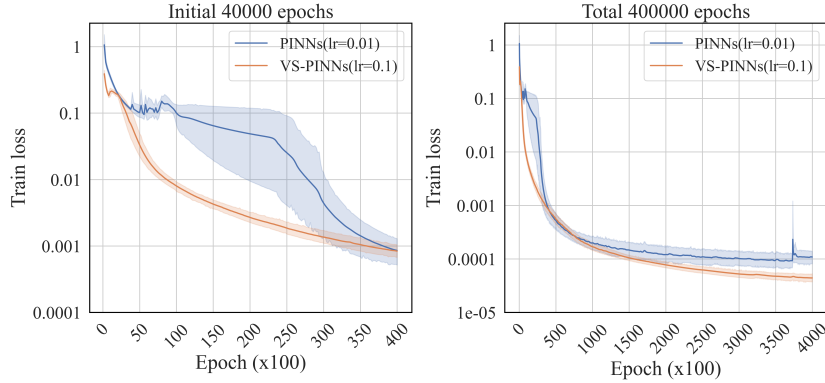

Figure 3: Loss of convection-diffusion equation trained by GD.

## D.2 Parameter settings for experiments

**Convection-diffusion equation** We conducted experiments on a convection-diffusion equation: We train 100,000 epochs of PINNs with $p = 3$ and VS-PINNs with $p = 2$, using the same settings as represented for the heat equation (509). Figures 3 and 4 show that VS-PINNs reach lower loss and achieve more stable convergence for both GD and Adam.

**Bi-harmonic equation** Consider the bi-harmonic equation

$$\begin{cases} \Delta^2 u\left(x,y\right) = f\left(x,y\right) & x, y \in [0,\pi] \\ u\left(x,0\right) = u\left(x,\pi\right) = u\left(0,y\right) = u\left(\pi,y\right) = 0 & x, y \in [0,\pi] \\ \frac{\partial}{\partial \mathbf{n}} u\left(x,0\right) = \frac{\partial}{\partial \mathbf{n}} u\left(x,\pi\right) = \frac{\partial}{\partial \mathbf{n}} u\left(0,y\right) = \frac{\partial}{\partial \mathbf{n}} u\left(\pi,y\right) = 0 & x, y \in [0,\pi], \end{cases} \quad (512)$$

where $f\left(x,y\right)$ are defined so that the exact solution is given by $u\left(x,y\right) = \frac{1}{5}\sin^2\left(x\right)y^2\left(\pi - y\right)^2$. We set $m = 1{,}000$, $n_o = 10{,}000$, $n_b = 400$, and the training collocation points are fixed once they are randomly selected. We experiment VS-PINNs with two cases: (i) $\phi_0 \approx u$, $\phi_1 \approx (u_{xx}, u_{yy})$ with $|\xi| = 2$ and $p = 3$ and (ii) the finest splitting of $\phi_0 \approx u$, $\phi_1 \approx \nabla u$, $\phi_2 \approx (u_{xx}, u_{yy})$, $\phi_3 \approx (u_{xxx}, u_{xxy}, u_{yyy})$ with $|\xi| = 1$ and $p = 2$. Regularization parameters are $\nu_1 = \nu_2 = \nu_3 = 1$ for derivative matching loss $\boldsymbol{\mu}$ and $\nu = 10$ for boundary loss $\boldsymbol{h}$. Figure 5 depicts the training loss of PINN and two VS-PINNs.

## E Additional Experimental Results

**Computational efficiency of VS-PINNs** Table 3 measures the GPU memory, running time, and the number of model parameters corresponding to experiments on elastic beam and bi-harmonic equations. Because VS-PINNs need as many networks as auxiliary variables, finer VS-PINN requires

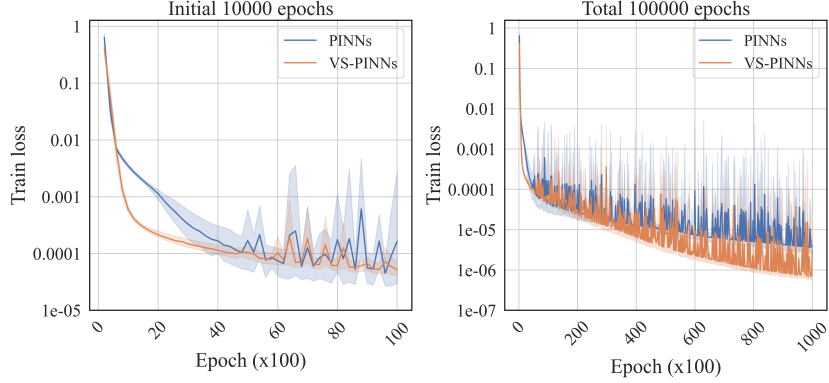

Figure 4: Loss of solving convection-diffusion equation trained by Adam.

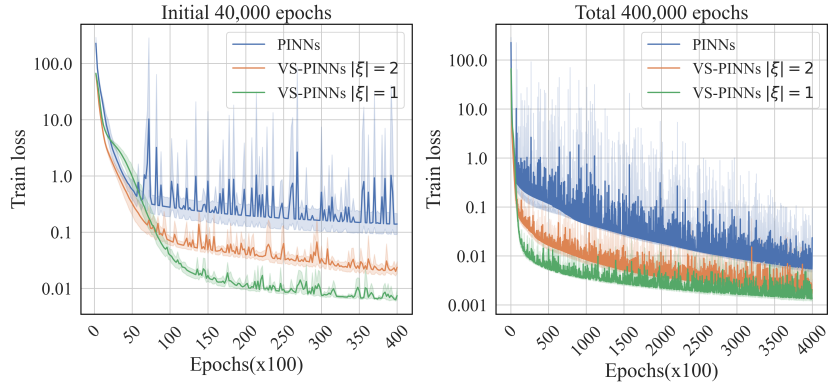

Figure 5: Loss comparison between PINNs with two VS-PINNs with $|\xi| = 1$ and 2.

more parameters to be trained. However, reducing the differentiation order in the Loss significantly reduces memory consumption, despite the additional loss terms.

**Error with true solution**    As the ultimate objective of PINNs is to approximate the solution to the PDE, we also quantified the discrepancy between the exact solution of the predicted solutions of the three models we tested. We tested five times for each PDE and splitting setup from different random seeds, and the averages of the Mean Squared Error (MSE) with the exact solution are reported in Table 4. We can observe that the results do not exhibit the same tendency as depicted in Figure 2. Since there is a discrepancy between optimization ability and the generalization error, it is not within the scope of this paper, and we believe that this is an important area for future research.

**GD results for fourth-order PDEs**    In order to address fourth-order PDEs (elastic beam and bi-harmonic equations), PINNs should employ $p = 5$ of the power of the ReLU$^p$ activation. When optimized with GD, this configuration results in a loss value of NaN, indicating that PINNs fail to train. Consequently, we present the results of our experiments with the Adam optimizer in Section 5. However, in contrast to PINNs, which exhibit difficulty in training, variable splitting demonstrates superior performance in terms of loss reduction for GD, as illustrated in Figure 6. This is attributed to the fact that variable splitting employs a lower derivative order and lower power $p$.

# F    Broader Impacts

We analyzed the behavior of PINNs when applied to general $k$th-order PDEs and provided theoretical understanding of the reasons why PINNs encounter difficulties in optimizing when dealing with high-order or high-dimensional problems. This paper has societal impact as it bridges artificial intelligence and physics, and hence has a wide range of potential applications. It could contribute to the advancement in scientific understanding, and foster scientific progress and innovation, benefiting

Table 3: Computation costs with width=1000 for experiments on the paper.

| PDE | Method | GPU memory | running time | parameters |
|---|---|---|---|---|
| Bi-harmonic | PINN | 801.052 Mb | 0.053 s/epoch | 4000 |
|  | VS-PINN $|\xi| = 2$ | 481.094 Mb | 0.049 s/epoch | 9000 |
|  | VS-PINN $|\xi| = 1$ | 81,466 Mb | 0.053 s/epoch | 20000 |
| Beam | PINN | 323.772 Mb | 0.037 s/epoch | 4000 |
|  | VS-PINN $|\xi| = 2$ | 240.689 Mb | 0.038 s/epoch | 9000 |
|  | VS-PINN $|\xi| = 1$ | 80.836 Mb | 0.040 s/epoch | 17000 |

Table 4: The average of Mean Square Error (MSE) with exact solution.

| PDE | PINNs | VS-PINNs $|\xi| = 2$ | VS-PINNs $|\xi| = 1$ |
|---|---|---|---|
| Elastic Beam | 3.0598E-08 | 2.2868E-08 | 4.3826E-07 |
| Bi-harmonic | 3.9319E-05 | 5.3055E-05 | 5.4418E-03 |

researchers in physics, mathematics, and artificial intelligence fields. By providing variable splitting which can address the convergence pathologies of PINNs, the paper suggests potential applications in industrial and technological fields, such as engineering, medicine, materials science, and environmental modeling, for complex system modeling, prediction, and optimization. Furthermore, as many problems have arisen in sciences and engineering tied with complex PDE systems, we expect that our work has the potential to be applicable in the enormous area such as climate forecasting, epidemics, molecular simulations, micro-mechanics, and modeling turbulent flows. Additionally, PDEs can be utilized in the development of military equipment. As with all numerical algorithms, however, it is not a work of developing a technique to go to warfare; rather, it is a tool for scientific inquiry. We encourage users of our model to focus on the positive impact of this work.

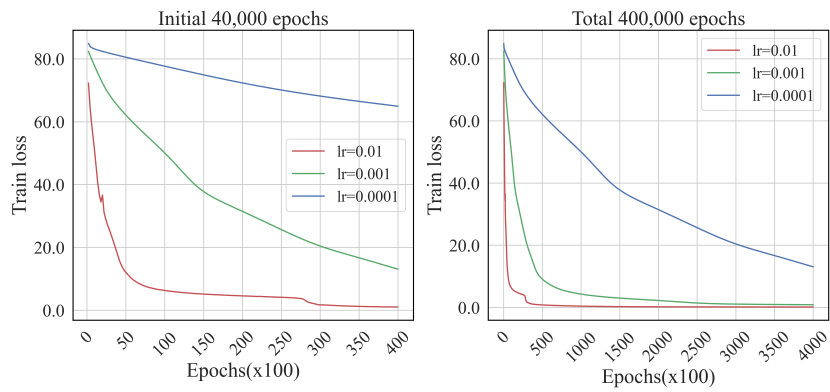

Figure 6: Loss of VS-PINNs trained by gradient descent with variant learning rates.

